# Provable Editing of Deep Neural Networks using Parametric Linear Relaxation

**Zhe Tao**
University of California, Davis
Davis, CA 95616, USA
zhetao@ucdavis.edu

**Aditya V. Thakur**
University of California, Davis
Davis, CA 95616, USA
avthakur@ucdavis.edu

## Abstract

Ensuring that a DNN satisfies a desired property is critical when deploying DNNs in safety-critical applications. There are efficient methods that can verify whether a DNN satisfies a property, as seen in the annual DNN verification competition (VNN-COMP). However, the problem of provably editing a DNN to satisfy a property remains challenging. We present PREPARED,[1] the first efficient technique for provable editing of DNNs. Given a DNN $\mathcal{N}$ with parameters $\theta$, input polytope P, and output polytope Q, PREPARED finds new parameters $\dot{\boldsymbol{\theta}}$ such that $\forall \mathbf{x} \in \mathrm{P}. \mathcal{N}(\mathbf{x}; \dot{\boldsymbol{\theta}}) \in \mathrm{Q}$ while minimizing the changes $\|\dot{\boldsymbol{\theta}} - \theta\|$. Given a DNN and a property it violates from the VNN-COMP benchmarks, PREPARED is able to provably edit the DNN to satisfy this property within 45 seconds. PREPARED is efficient because it relaxes the NP-hard provable editing problem to solving a linear program. The key contribution is the novel notion of Parametric Linear Relaxation, which enables PREPARED to construct tight output bounds of the DNN that are parameterized by the new parameters $\dot{\boldsymbol{\theta}}$. We demonstrate that PREPARED is more efficient and effective compared to prior DNN editing approaches **i)** using the VNN-COMP benchmarks, **ii)** by editing CIFAR10 and TINYIMAGENET image-recognition DNNs, and BERT sentiment-classification DNNs for local robustness, and **iii)** by training a DNN to model a geodynamics process and satisfy physics constraints.

## 1 Introduction

Ensuring that a DNN is correct and avoids harmful behaviors is critical when deploying them, especially in safety-critical contexts [8]. In such scenarios, it is vital to guarantee that a DNN satisfies a given property, e.g., a safety specification [35]. Towards this end, DNN verifiers aim to determine whether a DNN satisfies the given property [1]. There are efficient methods for DNN verification that can handle a wide-range of DNN architectures and properties, as seen in the annual DNN verification competition (VNN-COMP) [5]. However, the success of DNN verification reveals the next challenge: provably editing a DNN to satisfy a property. Formally, we define the provable editing problem as:

**Definition 1.1.** Given a DNN $\mathcal{N}$ with parameters $\theta$, an input polytope $\mathrm{P} \stackrel{\text{def}}{=} \{\mathbf{x} \mid \mathbf{Dx} \le \mathbf{e}\}$ and an output polytope $\mathrm{Q} \stackrel{\text{def}}{=} \{\mathbf{y} \mid \mathbf{Ay} \le \mathbf{b}\}$, the **provable editing problem** is to find new parameters $\dot{\boldsymbol{\theta}}$ that

$$\min \|\dot{\boldsymbol{\theta}} - \theta\| \quad \text{s.t.} \quad \forall \mathbf{x} \in \mathrm{P}. \mathcal{N}(\mathbf{x}; \dot{\boldsymbol{\theta}}) \in \mathrm{Q} \tag{1}$$

Provable editing can be used, for instance, to ensure that a classifier DNN is locally robust for an input $\mathbf{x}$. In this case, the input polytope $\mathrm{P} \stackrel{\text{def}}{=} \{\mathbf{x}' \mid \|\mathbf{x} - \mathbf{x}'\|_\infty \le \varepsilon\}$ is the $L^\infty$ ball around $\mathbf{x}$ with

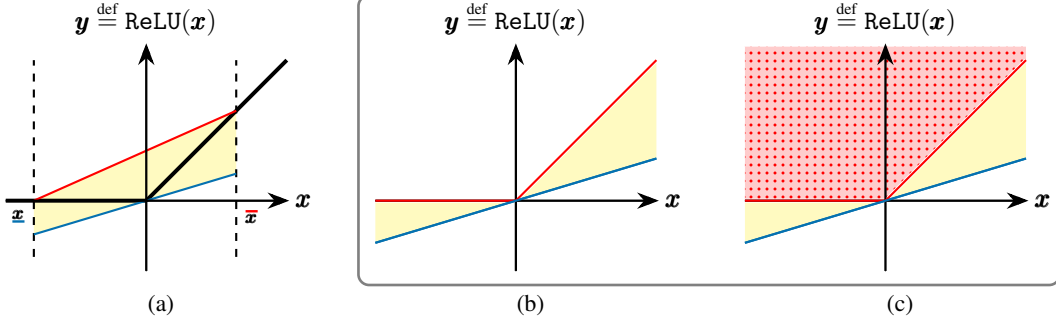

Figure 1: Linear relaxations for $y \overset{\text{def}}{=} \texttt{ReLU}(\boldsymbol{x})$. ── and ── denote the upper and lower bounds, ▢ denotes the relaxation. **(a)** Relaxation in prior approaches [49, 39], where the bounds are overapproximated using linear functions, and the upper bound is only sound within given input bounds $[\underline{\boldsymbol{x}}, \overline{\boldsymbol{x}}]$; **(b)** This work *exactly* represents the upper bound *for any* input upper bound variable $\ddot{\boldsymbol{x}}$, results in a *tighter* approximation; **(c)** Illustration of how this work *exactly* represents the upper bound $\ddot{\boldsymbol{y}}$ (──) by capturing the epigraph (▦) of $\texttt{ReLU}(\ddot{\boldsymbol{x}})$ with the linear constraint $\ddot{\boldsymbol{y}} \geq 0 \wedge \ddot{\boldsymbol{y}} \geq \ddot{\boldsymbol{x}}$. The lower bound $\dot{\boldsymbol{y}}$ (──) uses a linear relaxation $\dot{\boldsymbol{y}} \overset{\text{def}}{=} \boldsymbol{c}\dot{\boldsymbol{x}}$ with a constant $\boldsymbol{c} \in [0, 1]$.

radius $\varepsilon$, and the output polytope $\mathrm{Q} \overset{\text{def}}{=} \{\mathbf{y} \mid \bigwedge_{j \neq l} \mathbf{y}_j < \mathbf{y}_l\}$ requires that the output label is $l$; viz., $\arg\max \mathbf{y} = l$. Moreover, provable editing can be used either during training to guarantee that the DNN satisfies a property, or post-deployment to repair a given DNN.

The provable editing problem is challenging due to the presence of the universal quantifier; viz., finding new parameters such that *for all* inputs in the polytope P, the output of the edited network lies in the polytope Q. There are no existing approaches for efficiently and effectively solving the provable editing problem. Regularization-based approaches [26, 10, 28, 24] encode the property into the loss during training, but are unable to guarantee the property-satisfaction. SMT-based approaches [14, 16] provide guarantees, but are not efficient because they directly solve an NP-hard problem. Prior LP-based approaches [41, 42] are efficient because they can only handle a restricted class of provable editing problems; e.g., they assume that the vertices of the input polytope or the linear regions of the DNN can be efficiently enumerated, or the DNN architecture can be modified.

This paper presents `PREPARED`, the *first* **efficient** technique for **provable editing** of DNNs that runs in **polynomial time** (Theorem 3.4). Given a DNN and a property it violates from the `VNN-COMP'22` benchmarks, `PREPARED` is able to provably edit the DNN to satisfy this property within 45 seconds.

`PREPARED` approaches the problem of provable editing by constructing tight parametric bounds of the output $\mathcal{N}(\mathbf{x}; \dot{\boldsymbol{\theta}})$ for all inputs $\mathbf{x}$ in the input polytope P *in terms of the parameters* $\dot{\boldsymbol{\theta}}$, and then constraining these parametric bounds to lie within the output polytope Q. These parametric bounds are constructed compositionally per layer of the DNN, and are expressed as linear constraints, so that efficient Linear Programming (LP) solvers can be used to find an optimal solution.

Our *key insight* is to represent the parametric bounds indirectly via *underapproximations of the epigraph and hypograph* of the DNN layer. Consider the ReLU layer $y \overset{\text{def}}{=} \texttt{ReLU}(\boldsymbol{x})$. Prior approaches [49, 39] overapproximate its upper bound with a linear function (── in Figure 1(a)) using constant input bounds $[\underline{\boldsymbol{x}}, \overline{\boldsymbol{x}}]$. Such a linear relaxation is loose, and is only sound within the given constant input bounds $[\underline{\boldsymbol{x}}, \overline{\boldsymbol{x}}]$. Our approach is able to *exactly* represent the upper bound of ReLU output *for any* input upper bound variable $\ddot{\boldsymbol{x}}$ (── in Figure 1(b)). This is done by capturing the epigraph (▦ in Figure 1(c)) of $\texttt{ReLU}(\ddot{\boldsymbol{x}})$ with a linear constraint.

Our *key contribution* is a novel *Parametric Linear Relaxation* for DNN layers, which defines tight parametric bounds using linear constraints in terms of the layer parameters. Consider $\boldsymbol{v} \overset{\text{def}}{=} \boldsymbol{x}\dot{\boldsymbol{w}}$, which is a simplified representation of an Affine layer. We would like to construct bounds for all $\boldsymbol{x} \in [\underline{\boldsymbol{x}}, \overline{\boldsymbol{x}}]$ in terms of the variable parameter $\dot{\boldsymbol{w}}$. Prior approaches could achieve a sound, but loose linear relaxation that is not in terms of the layer parameters (Figure 2(a)). In contrast, our approach *exactly* represents the output bounds without any relaxation (Figure 2(b)) building upon our key insight of capturing the epigraph and hypograph of the DNN layer *in terms of the parameters* (Figure 2(c)).

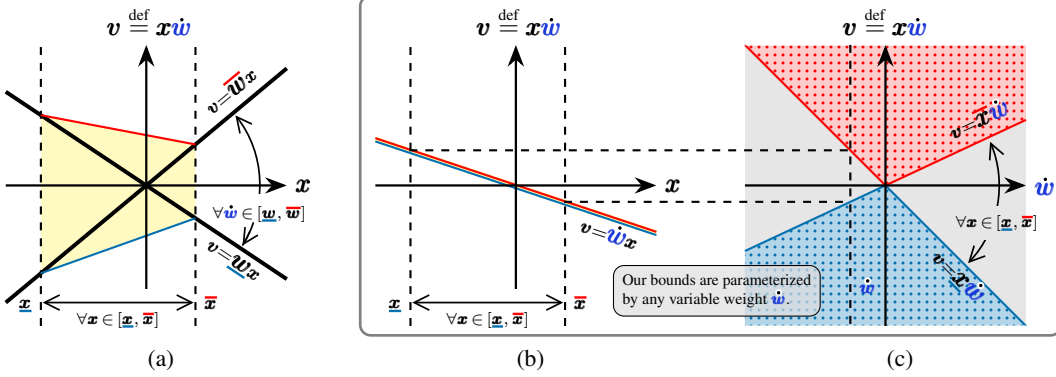

Figure 2: Linear relaxations for bounding $\boldsymbol{v} \overset{\text{def}}{=} \boldsymbol{x}\dot{\boldsymbol{w}}$ for all $\boldsymbol{x} \in [\underline{\boldsymbol{x}}, \overline{\boldsymbol{x}}]$, with *variable parameter* $\dot{\boldsymbol{w}}$. ──── and ──── denote the upper and lower bound overapproximations. ▨ denotes the relaxation. **(a)** A loose relaxation for all $\boldsymbol{x} \in [\underline{\boldsymbol{x}}, \overline{\boldsymbol{x}}]$ and for all $\dot{\boldsymbol{w}} \in [\underline{\boldsymbol{w}}, \overline{\boldsymbol{w}}]$, where $\dot{\boldsymbol{w}}$ is not treated as a parameter but another input within constant bounds $[\underline{\boldsymbol{w}}, \overline{\boldsymbol{w}}]$. **(b)** This work *exactly* represents the bounds for all inputs $\boldsymbol{x} \in [\underline{\boldsymbol{x}}, \overline{\boldsymbol{x}}]$, parameterized by variable $\dot{\boldsymbol{w}}$, *without any relaxation*. Here we plot a case when $\dot{\boldsymbol{w}} < 0$, while our parametric bounds are exact for any $\dot{\boldsymbol{w}}$. **(c)** Illustration of how our parametric bounds are defined for the function $\boldsymbol{v} \overset{\text{def}}{=} \boldsymbol{x}\dot{\boldsymbol{w}}$ in terms of the variable parameter $\dot{\boldsymbol{w}}$. ▦ denotes the *true output region* for all slope (input) $\boldsymbol{x} \in [\underline{\boldsymbol{x}}, \overline{\boldsymbol{x}}]$. The upper bound $\overline{\dot{\boldsymbol{v}}}$ (────) is exactly captured by the epigraph (▨), defined by linear constraint ($\overline{\dot{\boldsymbol{v}}} \geq \underline{\boldsymbol{x}}\dot{\boldsymbol{w}} \wedge \overline{\dot{\boldsymbol{v}}} \geq \overline{\boldsymbol{x}}\dot{\boldsymbol{w}}$); the lower bound $\underline{\dot{\boldsymbol{v}}}$ (────) is exactly captured by the hypograph (▨), defined by linear constraint ($\underline{\dot{\boldsymbol{v}}} \leq \underline{\boldsymbol{x}}\dot{\boldsymbol{w}} \wedge \underline{\dot{\boldsymbol{v}}} \leq \overline{\boldsymbol{x}}\dot{\boldsymbol{w}}$).

We evaluate PREPARED **i)** using the VNN-COMP benchmarks, **ii)** by editing CIFAR10 and TINYIM-AGENET image-recognition DNNs, and BERT sentiment-classification DNNs for local robustness, and **iii)** by training a DNN to model a geodynamics process and satisfy physics constraints. Because there were no prior efficient approaches for provable editing, we implemented new *provable fine-tuning* baselines by integrating prior DNN editing approaches with a verifier in the loop. PREPARED outperformed such baselines demonstrating its effectiveness and efficiency.

## 2 Related work

**Regularized training** [20, 45, 30, 10, 19, 22, 47] incorporates constraints as regularization into the training loss, but does not guarantee constraint-satisfaction in the trained DNN. For the DNN editing problem we are addressing, which involves universally-quantified logical formulas, DL2 [10] is the state-of-the-art regularized training method.

**Certified training** [26, 15, 36, 2, 31, 28, 24] is a type of regularized training geared towards adversarial robustness. SABR [28] and STAPS [24] are state-of-the-art certified training approaches. However, they also do not guarantee constraint-satisfaction.

**SMT-based editing approaches** [14, 23, 16] directly solve the NP-hard provable editing problem using an SMT solver. Thus, they are inefficient and do not scale beyond small DNNs. For instance, the recent DeepSaDe approach [16] incorporates an SMT solver during training, but only uses the SMT solver to edit the last layer. However, it still can take 500× longer than regularized training.

**LP-based editing approaches** [41, 42] relax the provable editing problem to solving an LP problem, which makes them efficient. However, prior approaches can only handle a restricted class of provable editing problems. The state-of-the-art provable editing approach APRNN [42] is efficient for editing input points. However, it does not scale to the general provable editing problem of Definition 1.1 due to the need for enumerating vertices of the input polytope. PRDNN [41] was the precursor to APRNN and suffers from similar limitations; moreover, PRDNN modifies the architecture of DNN and requires enumeration of linear regions to edit an input polytope. REASSURE [12] repairs a linear region of a DNN by adding a patch DNN for each such linear region. Thus, it also requires enumeration of linear regions to edit an input polytope. Further, it has been shown to be unsound and incomplete (Section 5 of [42]), and suffers from significant runtime and memory overheads (Section 7.8 of [42]).

**DNN verification** [38, 39, 49, 44, 37, 48, 46, 9, 4] aims to determine whether a DNN satisfies a given property. As shown in Section 4, provable editing can be used when a verifier determines that a DNN violates a property. Most verification techniques use linear relaxations to bound the DNN output. However, these linear relaxations cannot be used for provable editing because they are not parameterized in terms of the DNN parameters. Instead, verification techniques assume that the parameters are constant and focus on designing linear relaxations for activation layers like ReLU [34, 43]. However, it is not clear how even these activation layer relaxations could be used for provable editing. For instance, these relaxations require constant input bounds; loose constant input bounds result in loose linear relaxations. In the context of provable editing, if the ReLU layer follows an Affine layer with editable parameters, then the constant input bounds to the ReLU layer would be very loose (e.g., $[-\infty, \infty]$) in order to be sound for all possible edits. Hence, the resulting linear relaxation would also be loose. In contrast, our technique does not require constant input bounds for any layer, and produces an exact upper bound for the ReLU layer (Theorem 3.9). Applying our Parametric Linear Relaxation to DNN verification remains future work.

## 3 Provable editing of DNNs using Parametric Linear Relaxation

We use $x \in \mathbb{R}$ to denote a scalar, $\mathbf{x} \in \mathbb{R}^m$ to denote a column vector, and $\mathbf{W} \in \mathbb{R}^{n \times m}$ to denote a matrix. Blue-colored variables with a dot denote LP decision variables, e.g., $\dot{\mathbf{x}}, \ddot{\mathbf{x}}, \dot{\mathbf{W}}, \dot{\mathbf{b}}$, etc. The proofs for all theorems can be found in Appendix A.

**Definition 3.1** (DNN). Let $\mathbf{y} \stackrel{\text{def}}{=} \mathcal{N}(\mathbf{x}; \theta)$ denote an $L$-layer deep neural network (DNN) $\mathcal{N}$ with parameters $\theta$, input $\mathbf{x}$ and output $\mathbf{y}$. The DNN is defined iteratively as $\mathbf{x}^{(\ell+1)} \stackrel{\text{def}}{=} \mathcal{N}^{(\ell)}(\mathbf{x}^{(\ell)}; \theta^{(\ell)})$ for each layer $\mathcal{N}^{(\ell)}$ with parameters $\theta^{(\ell)}$, where $\mathbf{x}^{(0)} \stackrel{\text{def}}{=} \mathbf{x}$, $\mathbf{y} \stackrel{\text{def}}{=} \mathbf{x}^{(L)}$ and $0 \leq \ell < L$. ∎

For ease of exposition, we defer our approach for the general provable editing problem of Definition 1.1 to Appendix B. In this section, we consider the following provable interval editing problem, where the input and output constraints are constant interval bounds:

**Definition 3.2.** Given a DNN $\mathcal{N}$ with parameters $\theta$, constant input bounds $[\underline{\mathbf{x}}, \overline{\mathbf{x}}]$ and constant output bounds $[\underline{\mathbf{y}}, \overline{\mathbf{y}}]$, the *provable interval editing problem* is to find new parameters $\dot{\theta}$ that

$$\min \|\dot{\theta} - \theta\| \quad \text{s.t.} \quad \forall \mathbf{x} \in [\underline{\mathbf{x}}, \overline{\mathbf{x}}]. \ \mathcal{N}(\mathbf{x}; \dot{\theta}) \in [\underline{\mathbf{y}}, \overline{\mathbf{y}}] \tag{2} ∎$$

### 3.1 Parametric Linear Relaxation

Consider a DNN $\mathcal{N}$ with variable parameters $\dot{\theta}$ and input bounds $[\dot{\mathbf{x}}, \ddot{\mathbf{x}}]$, our goal is to construct sound parametric output bounds $[\dot{\mathbf{y}}, \ddot{\mathbf{y}}]$ of $\mathcal{N}$ in terms of $\dot{\theta}$ and $[\dot{\mathbf{x}}, \ddot{\mathbf{x}}]$. The exact definition is

$$\dot{\mathbf{y}} \leq \min\left\{\mathcal{N}(\mathbf{x}; \dot{\theta}) \mid \dot{\mathbf{x}} \leq \mathbf{x} \leq \ddot{\mathbf{x}}\right\} \wedge \ddot{\mathbf{y}} \geq \max\left\{\mathcal{N}(\mathbf{x}; \dot{\theta}) \mid \dot{\mathbf{x}} \leq \mathbf{x} \leq \ddot{\mathbf{x}}\right\} \tag{3}$$

However, this universally-quantified non-linear formula is expensive to solve [6, 7]. Our key contribution is a novel *Parametric Linear Relaxation*, a *poly-size linear formula* that implies Equation 3.

**Definition 3.3.** The *Parametric Linear Relaxation* $\overline{\varphi}_{\mathcal{N}}(\dot{\mathbf{y}}, \ddot{\mathbf{y}}, \dot{\mathbf{x}}, \ddot{\mathbf{x}}; \dot{\theta})$ for $\mathcal{N}$ is a poly-size linear formula that implies $\forall \mathbf{x} \in [\dot{\mathbf{x}}, \ddot{\mathbf{x}}]. \ \mathcal{N}(\mathbf{x}; \dot{\theta}) \in [\dot{\mathbf{y}}, \ddot{\mathbf{y}}]$. ∎

In other words, any satisfying assignment $(\underline{\mathbf{y}}, \overline{\mathbf{y}}, \underline{\mathbf{x}}, \overline{\mathbf{x}}, \theta)$ of the formula $\overline{\varphi}_{\mathcal{N}}(\dot{\mathbf{y}}, \ddot{\mathbf{y}}, \dot{\mathbf{x}}, \ddot{\mathbf{x}}; \dot{\theta})$ yields sound output bounds $[\underline{\mathbf{y}}, \overline{\mathbf{y}}]$ for $\mathcal{N}$, viz., $\forall \mathbf{x} \in [\underline{\mathbf{x}}, \overline{\mathbf{x}}]. \ \mathcal{N}(\mathbf{x}; \theta) \in [\underline{\mathbf{y}}, \overline{\mathbf{y}}]$. The size of the formula $\overline{\varphi}_{\mathcal{N}}(\dot{\mathbf{y}}, \ddot{\mathbf{y}}, \dot{\mathbf{x}}, \ddot{\mathbf{x}}; \dot{\theta})$ is polynomial in the size of the DNN, i.e., the number of parameters, layers, and the input and output dimensions of each layer.

### 3.2 Provable editing via Parametric Linear Relaxation

The following theorem shows how Parametric Linear Relaxation can be used to solve the provable interval editing problem.

**Theorem 3.4.** *Given a provable interval editing problem (Definition 3.2) for DNN $\mathcal{N}$ and parameters $\theta$ with input bounds $[\underline{\mathbf{x}}, \overline{\mathbf{x}}]$ and output bounds $[\underline{\mathbf{y}}, \overline{\mathbf{y}}]$, let $\overline{\varphi}_{\mathcal{N}}(\underline{\dot{\mathbf{y}}}, \overline{\dot{\mathbf{y}}}, \underline{\mathbf{x}}, \overline{\mathbf{x}}; \dot{\theta})$ be a Parametric Linear Relaxation for $\mathcal{N}$. Then the following linear program can be solved in polynomial time in the size of the DNN $\mathcal{N}$, and whose solution is a solution to the provable interval editing problem:*

$$\min \|\dot{\theta} - \theta\| \quad s.t. \quad \overline{\varphi}_{\mathcal{N}}(\underline{\dot{\mathbf{y}}}, \overline{\dot{\mathbf{y}}}, \underline{\mathbf{x}}, \overline{\mathbf{x}}; \dot{\theta}) \wedge (\underline{\mathbf{y}} \leq \underline{\dot{\mathbf{y}}} \wedge \overline{\dot{\mathbf{y}}} \leq \overline{\mathbf{y}}) \tag{4}$$

### 3.3 Compositional construction of Parametric Linear Relaxation

**Definition 3.5.** Given an $L$-layer DNN $\mathcal{N}$ with parameters $\dot{\theta}$, constant input bounds $[\underline{\mathbf{x}}, \overline{\mathbf{x}}]$ and Parametric Linear Relaxation $\overline{\varphi}_{\mathcal{N}^{(\ell)}}$ for each layer $\mathcal{N}^{(\ell)}$, we define the *compositional Parametric Linear Relaxation* $\overline{\varphi}_{\mathcal{N}}(\underline{\dot{\mathbf{y}}}, \overline{\dot{\mathbf{y}}}, \underline{\mathbf{x}}, \overline{\mathbf{x}}; \dot{\theta})$ as follows, where $[\underline{\dot{\mathbf{y}}}, \overline{\dot{\mathbf{y}}}] \stackrel{\text{def}}{=} [\underline{\dot{\mathbf{x}}}^{(L)}, \overline{\dot{\mathbf{x}}}^{(L)}]$ and $[\underline{\dot{\mathbf{x}}}^{(0)}, \overline{\dot{\mathbf{x}}}^{(0)}] \stackrel{\text{def}}{=} [\underline{\mathbf{x}}, \overline{\mathbf{x}}]$:

$$\overline{\varphi}_{\mathcal{N}}(\underline{\dot{\mathbf{y}}}, \overline{\dot{\mathbf{y}}}, \underline{\mathbf{x}}, \overline{\mathbf{x}}; \dot{\theta}) \stackrel{\text{def}}{=} \bigwedge_{0 \leq \ell < L} \overline{\varphi}_{\mathcal{N}^{(\ell)}}(\underline{\dot{\mathbf{x}}}^{(\ell+1)}, \overline{\dot{\mathbf{x}}}^{(\ell+1)}, \underline{\dot{\mathbf{x}}}^{(\ell)}, \overline{\dot{\mathbf{x}}}^{(\ell)}; \dot{\theta}^{(\ell)}) \tag{5} \blacksquare$$

**Theorem 3.6.** *Definition 3.5 is a Parametric Linear Relaxation for the DNN $\mathcal{N}$.*

By Theorems 3.4 and 3.6, we see that to solve the provable editing problem, we just need to construct Parametric Linear Relaxations $\overline{\varphi}_{\mathcal{N}^{(\ell)}}(\underline{\dot{\mathbf{y}}}, \overline{\dot{\mathbf{y}}}, \underline{\dot{\mathbf{x}}}, \overline{\dot{\mathbf{x}}}; \dot{\theta})$ *for each DNN layer $\mathcal{N}^{(\ell)}$*. In practice, $\overline{\varphi}_{\mathcal{N}^{(\ell)}}(\underline{\dot{\mathbf{y}}}, \overline{\dot{\mathbf{y}}}, \underline{\dot{\mathbf{x}}}, \overline{\dot{\mathbf{x}}}; \dot{\theta})$ is defined by separate formulas $\overline{\varphi}_{\mathcal{N}^{(\ell)}}(\overline{\dot{\mathbf{y}}}, \underline{\dot{\mathbf{x}}}, \overline{\dot{\mathbf{x}}}; \dot{\theta})$ and $\underline{\varphi}_{\mathcal{N}^{(\ell)}}(\underline{\dot{\mathbf{y}}}, \underline{\dot{\mathbf{x}}}, \overline{\dot{\mathbf{x}}}; \dot{\theta})$ for the upper and lower bounds, respectively: $\overline{\varphi}_{\mathcal{N}^{(\ell)}}(\underline{\dot{\mathbf{y}}}, \overline{\dot{\mathbf{y}}}, \underline{\dot{\mathbf{x}}}, \overline{\dot{\mathbf{x}}}; \dot{\theta}) \stackrel{\text{def}}{=} \overline{\varphi}_{\mathcal{N}^{(\ell)}}(\overline{\dot{\mathbf{y}}}, \underline{\dot{\mathbf{x}}}, \overline{\dot{\mathbf{x}}}; \dot{\theta}) \wedge \underline{\varphi}_{\mathcal{N}^{(\ell)}}(\underline{\dot{\mathbf{y}}}, \underline{\dot{\mathbf{x}}}, \overline{\dot{\mathbf{x}}}; \dot{\theta})$. For brevity, we describe how to construct Parametric Linear Relaxation for ReLU and Affine layers. Appendix C presents Parametric Linear Relaxations for Tanh, Sigmoid and ELU layers.

### 3.4 Parametric Linear Relaxation for ReLU layer

**Definition 3.7.** $\mathbf{y} \stackrel{\text{def}}{=} \text{ReLU}(\mathbf{x})$ with input $\mathbf{x} \in \mathbb{R}^m$ and output $\mathbf{y} \in \mathbb{R}^m$ is defined as $\mathbf{y} \stackrel{\text{def}}{=} \max\{\mathbf{x}, 0\}$.

**Definition 3.8.** For a ReLU layer with variable input bounds $[\underline{\dot{\mathbf{x}}}, \overline{\dot{\mathbf{x}}}]$, its Parametric Linear Relaxation is defined as $\overline{\varphi}_{\text{ReLU}}(\underline{\dot{\mathbf{y}}}, \overline{\dot{\mathbf{y}}}, \underline{\dot{\mathbf{x}}}, \overline{\dot{\mathbf{x}}}) \stackrel{\text{def}}{=} \overline{\varphi}_{\text{ReLU}}(\overline{\dot{\mathbf{y}}}, \overline{\dot{\mathbf{x}}}) \wedge \underline{\varphi}_{\text{ReLU}}(\underline{\dot{\mathbf{y}}}, \underline{\dot{\mathbf{x}}})$ where:

$$\overline{\varphi}_{\text{ReLU}}(\overline{\dot{\mathbf{y}}}, \overline{\dot{\mathbf{x}}}) \stackrel{\text{def}}{=} \overline{\dot{\mathbf{y}}} \geq \overline{\dot{\mathbf{x}}} \wedge \overline{\dot{\mathbf{y}}} \geq 0 \qquad \underline{\varphi}_{\text{ReLU}}(\underline{\dot{\mathbf{y}}}, \underline{\dot{\mathbf{x}}}) \stackrel{\text{def}}{=} \underline{\dot{\mathbf{y}}} = \mathbf{c}\underline{\dot{\mathbf{x}}} \tag{6}$$

and $\mathbf{c} \in \mathbb{R}^m$ are constants such that $0 \leq \mathbf{c}_i \leq 1$. $\blacksquare$

Because ReLU is a convex function, as seen in Figure 1(c), the upper bound constraint $\overline{\dot{\mathbf{y}}}_i \geq \overline{\dot{\mathbf{x}}}_i \wedge \overline{\dot{\mathbf{y}}}_i \geq 0$ for each $\overline{\dot{\mathbf{y}}}_i$ exactly captures the epigraph (▦) of $\text{ReLU}(\overline{\dot{\mathbf{x}}}_i)$. For each lower bound $\underline{\dot{\mathbf{y}}}_i$, we use a linear relaxation $\underline{\dot{\mathbf{y}}}_i = \mathbf{c}_i \underline{\dot{\mathbf{x}}}_i$ (——) of $\text{ReLU}(\underline{\dot{\mathbf{x}}})_i$ with a constant $0 \leq \mathbf{c}_i \leq 1$.

**Theorem 3.9.** *Definition 3.8 is a Parametric Linear Relaxation for the ReLU layer that captures the exact upper bound.*

### 3.5 Parametric Linear Relaxation for Affine layer

**Definition 3.10.** $\mathbf{y} \stackrel{\text{def}}{=} \text{Affine}(\mathbf{x}; \mathbf{W}, \mathbf{b})$ with input $\mathbf{x} \in \mathbb{R}^m$, output $\mathbf{y} \in \mathbb{R}^n$, weight $\mathbf{W} \in \mathbb{R}^{n \times m}$ and bias $\mathbf{b} \in \mathbb{R}^n$ is defined as $\mathbf{y} \stackrel{\text{def}}{=} \mathbf{W}\mathbf{x} + \mathbf{b}$, which expands to $\mathbf{y}_j \stackrel{\text{def}}{=} \sum_{i=0}^{m} \mathbf{x}_i \mathbf{W}_{ji} + \mathbf{b}_j$.

In DNN verification, bounding Affine layers is straightforward, because the parameters are constants. However, in provable editing, this is challenging, because the parameters are variables. In particular, we need to bound the multiplication $\boldsymbol{x}\dot{\boldsymbol{w}}$ between the universally-quantified input $\boldsymbol{x} \in [\underline{\boldsymbol{x}}, \overline{\boldsymbol{x}}]$ and the variable weight $\dot{\boldsymbol{w}}$. As seen in Figure 2(a), if the bounds for $\boldsymbol{x}\dot{\boldsymbol{w}}$ are not parameterized by $\dot{\boldsymbol{w}}$, a linear relaxation would require constant bounds $[\underline{\boldsymbol{w}}, \overline{\boldsymbol{w}}]$ for $\dot{\boldsymbol{w}}$, resulting in loose bounds. Our approach constructs tight parametric output bounds using Parametric Linear Relaxation. We now describe our approach for an Affine layer **i)** with constant input bounds $[\underline{\mathbf{x}}, \overline{\mathbf{x}}]$ when it is the first layer of a DNN, and **ii)** with variable input bounds $[\underline{\dot{\mathbf{x}}}, \overline{\dot{\mathbf{x}}}]$ when it is not.

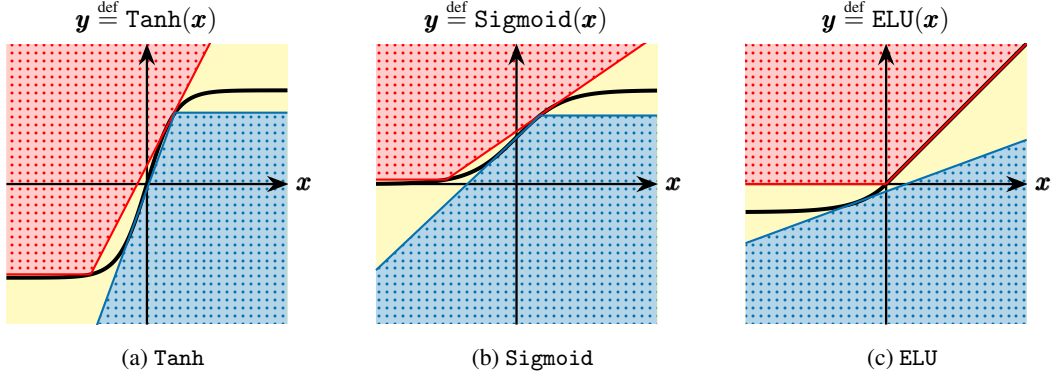

$$y \stackrel{\text{def}}{=} \texttt{Tanh}(\boldsymbol{x}) \qquad y \stackrel{\text{def}}{=} \texttt{Sigmoid}(\boldsymbol{x}) \qquad y \stackrel{\text{def}}{=} \texttt{ELU}(\boldsymbol{x})$$

(a) Tanh (b) Sigmoid (c) ELU

Figure 3: Illustration of Parameterized Linear Relaxations for Tanh, Sigmoid and ELU. ——— and ——— denote the upper and lower bound approximations; ░░░ and ░░░ denotes the convex underapproximations $\overline{\varphi}(\ddot{\boldsymbol{y}}, \ddot{\boldsymbol{x}})$ and $\underline{\varphi}(\dot{\boldsymbol{y}}, \dot{\boldsymbol{x}})$ for the epigraph and hypograph; ▢ denotes relaxation.

### 3.5.1 Affine layer with constant input bounds

**Definition 3.11.** For an Affine layer with constant input bounds $[\underline{\mathbf{x}}, \overline{\mathbf{x}}]$ and variable parameters $\dot{\mathbf{W}}, \dot{\mathbf{b}}$, its Parametric Linear Relaxation is defined as $\overline{\underline{\varphi}}_{\texttt{Aff}}(\dot{\underline{\boldsymbol{y}}}, \ddot{\boldsymbol{y}}, \underline{\mathbf{x}}, \overline{\mathbf{x}}, \dot{\mathbf{W}}, \dot{\mathbf{b}}) \stackrel{\text{def}}{=} \overline{\varphi}_{\texttt{Aff}}(\ddot{\boldsymbol{y}}, \underline{\mathbf{x}}, \overline{\mathbf{x}}, \dot{\mathbf{W}}, \dot{\mathbf{b}}) \wedge$ $\underline{\varphi}_{\texttt{Aff}}(\dot{\underline{\boldsymbol{y}}}, \underline{\mathbf{x}}, \overline{\mathbf{x}}, \dot{\mathbf{W}}, \dot{\mathbf{b}})$ where:

$$\overline{\varphi}_{\texttt{Aff}}(\ddot{\boldsymbol{y}}, \underline{\mathbf{x}}, \overline{\mathbf{x}}, \dot{\mathbf{W}}, \dot{\mathbf{b}}) \stackrel{\text{def}}{=} \bigwedge_j \ddot{\boldsymbol{y}}_j = \sum_i \ddot{\mathbf{V}}_{ji} + \dot{\mathbf{b}}_j \ \wedge \ \ddot{\mathbf{V}} \geq \dot{\mathbf{W}}\underline{\mathbf{x}} \ \wedge \ \ddot{\mathbf{V}} \geq \dot{\mathbf{W}}\overline{\mathbf{x}}$$

$$\underline{\varphi}_{\texttt{Aff}}(\dot{\underline{\boldsymbol{y}}}, \underline{\mathbf{x}}, \overline{\mathbf{x}}, \dot{\mathbf{W}}, \dot{\mathbf{b}}) \stackrel{\text{def}}{=} \bigwedge_j \dot{\underline{\boldsymbol{y}}}_j = \sum_i \dot{\underline{\mathbf{V}}}_{ji} + \dot{\mathbf{b}}_j \ \wedge \ \dot{\underline{\mathbf{V}}} \leq \dot{\mathbf{W}}\underline{\mathbf{x}} \ \wedge \ \dot{\underline{\mathbf{V}}} \leq \dot{\mathbf{W}}\overline{\mathbf{x}}$$

(7) ■

The core of this definition is constructing parametric bounds $[\dot{\underline{\boldsymbol{v}}}, \ddot{\boldsymbol{v}}]$ for the multiplication $\boldsymbol{x}\dot{\boldsymbol{w}}$ for all $\boldsymbol{x} \in [\underline{\boldsymbol{x}}, \overline{\boldsymbol{x}}]$ in terms of the variable weight $\dot{\boldsymbol{w}}$. As seen in Figure 2(c), for all $\boldsymbol{x} \in [\underline{\boldsymbol{x}}, \overline{\boldsymbol{x}}]$, the upper bound (———) of $\boldsymbol{x}\dot{\boldsymbol{w}}$ is defined by a piecewise-linear convex function $\overline{f}(\dot{\boldsymbol{w}}) \stackrel{\text{def}}{=} \max\{\underline{\boldsymbol{x}}\dot{\boldsymbol{w}}, \overline{\boldsymbol{x}}\dot{\boldsymbol{w}}\}$. Hence, the upper bound constraint $\ddot{\boldsymbol{v}} \geq \underline{\boldsymbol{x}}\dot{\boldsymbol{w}} \wedge \ddot{\boldsymbol{v}} \geq \overline{\boldsymbol{x}}\dot{\boldsymbol{w}}$ *exactly* captures the epigraph (░░░) of $\overline{f}(\dot{\boldsymbol{w}})$. Similarly, the lower bound (———) of $\boldsymbol{x}\dot{\boldsymbol{w}}$ is defined by a piecewise-linear concave function $\underline{f}(\dot{\boldsymbol{w}}) \stackrel{\text{def}}{=} \min\{\underline{\boldsymbol{x}}\dot{\boldsymbol{w}}, \overline{\boldsymbol{x}}\dot{\boldsymbol{w}}\}$. Hence, the lower bound constraint $\dot{\underline{\boldsymbol{v}}} \leq \underline{\boldsymbol{x}}\dot{\boldsymbol{w}} \wedge \dot{\underline{\boldsymbol{v}}} \leq \overline{\boldsymbol{x}}\dot{\boldsymbol{w}}$ *exactly* captures the hypograph (░░░) of $\underline{f}(\dot{\boldsymbol{w}})$.

**Theorem 3.12.** *Definition 3.11 is a Parametric Linear Relaxation for the Affine layer with constant input bounds $[\underline{\mathbf{x}}, \overline{\mathbf{x}}]$ that captures the exact lower and upper bounds.*

### 3.5.2 Affine layer with variable input bounds

We freeze the parameter weight $\dot{\mathbf{W}}$ to be constant $\mathbf{W}$ (e.g., the original weight) when the input bounds are variable to avoid non-linearity.

**Definition 3.13.** For an Affine layer with variable input bounds $[\dot{\underline{\mathbf{x}}}, \ddot{\mathbf{x}}]$, constant weight $\mathbf{W}$ and variable bias $\dot{\mathbf{b}}$, its Parametric Linear Relaxation is defined as $\overline{\underline{\varphi}}_{\texttt{Aff}}(\dot{\underline{\boldsymbol{y}}}, \ddot{\boldsymbol{y}}, \dot{\underline{\mathbf{x}}}, \ddot{\mathbf{x}}, \mathbf{W}, \dot{\mathbf{b}}) \stackrel{\text{def}}{=} \overline{\varphi}_{\texttt{Aff}}(\ddot{\boldsymbol{y}}, \dot{\underline{\mathbf{x}}}, \ddot{\mathbf{x}}, \mathbf{W}, \dot{\mathbf{b}}) \wedge$ $\underline{\varphi}_{\texttt{Aff}}(\dot{\underline{\boldsymbol{y}}}, \dot{\underline{\mathbf{x}}}, \ddot{\mathbf{x}}, \mathbf{W}, \dot{\mathbf{b}})$ where:

$$\overline{\varphi}_{\texttt{Aff}}(\ddot{\boldsymbol{y}}, \dot{\underline{\mathbf{x}}}, \ddot{\mathbf{x}}, \mathbf{W}, \dot{\mathbf{b}}) \stackrel{\text{def}}{=} \bigwedge_j \ddot{\boldsymbol{y}}_j = \sum_i \overline{\mathbf{V}}_{ji} + \dot{\mathbf{b}}_j \qquad \text{where} \begin{cases} \underline{\mathbf{V}}_{ji} \stackrel{\text{def}}{=} \dot{\underline{\mathbf{x}}}_j \mathbf{W}_{ji} \text{ and } \overline{\mathbf{V}}_{ji} \stackrel{\text{def}}{=} \ddot{\mathbf{x}}_i \mathbf{W}_{ji} \ \text{ if } \ \mathbf{W}_{ji} \geq 0 \\ \underline{\mathbf{V}}_{ji} \stackrel{\text{def}}{=} \ddot{\mathbf{x}}_j \mathbf{W}_{ji} \text{ and } \overline{\mathbf{V}}_{ji} \stackrel{\text{def}}{=} \dot{\underline{\mathbf{x}}}_i \mathbf{W}_{ji} \ \text{ otherwise} \end{cases}$$

$$\underline{\varphi}_{\texttt{Aff}}(\dot{\underline{\boldsymbol{y}}}, \dot{\underline{\mathbf{x}}}, \ddot{\mathbf{x}}, \mathbf{W}, \dot{\mathbf{b}}) \stackrel{\text{def}}{=} \bigwedge_j \dot{\underline{\boldsymbol{y}}}_j = \sum_i \underline{\mathbf{V}}_{ji} + \dot{\mathbf{b}}_j$$

(8) ■

Using constant weight $\mathbf{W}$ avoids non-linearity due to the multiplication $\boldsymbol{x}\boldsymbol{w}$ where $\boldsymbol{x}$ is universally-quantified in variable bounds $[\dot{\underline{\boldsymbol{x}}}, \ddot{\boldsymbol{x}}]$. Thus, the bounds $[\underline{\boldsymbol{v}}, \overline{\boldsymbol{v}}]$ of $\boldsymbol{x}\boldsymbol{w}$ for all $\boldsymbol{x} \in [\dot{\underline{\boldsymbol{x}}}, \ddot{\boldsymbol{x}}]$ are determined by the sign of the constant $\boldsymbol{w}$; viz., $[\underline{\boldsymbol{v}}, \overline{\boldsymbol{v}}] \stackrel{\text{def}}{=} [\dot{\underline{\boldsymbol{x}}}\boldsymbol{w}, \ddot{\boldsymbol{x}}\boldsymbol{w}]$ if $\boldsymbol{w} \geq 0$, otherwise $[\underline{\boldsymbol{v}}, \overline{\boldsymbol{v}}] \stackrel{\text{def}}{=} [\ddot{\boldsymbol{x}}\boldsymbol{w}, \dot{\underline{\boldsymbol{x}}}\boldsymbol{w}]$.

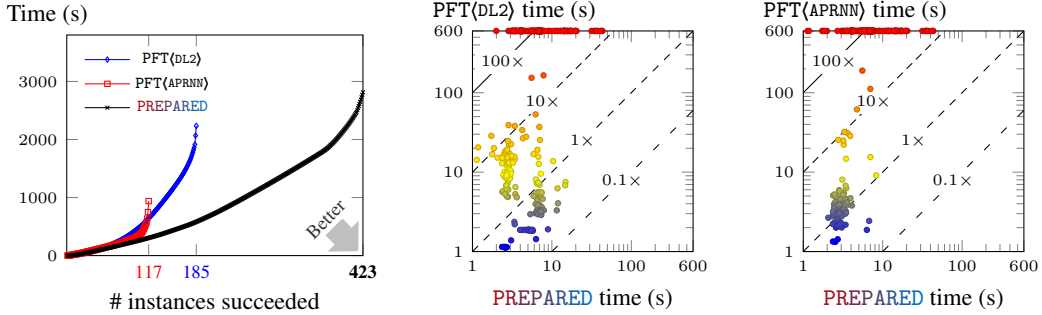

(a) **Left**: cactus plot of runtime for 423 single-property editing problems, with 600 seconds time limit for each problem. PREPARED succeeded on *all* 423 problems, while PFT⟨DL2⟩ succeeded on 185, and PFT⟨APRNN⟩ succeeded on 117. **Middle**: speed up for PREPARED vs PFT⟨DL2⟩. **Right**: speed up for PREPARED vs PFT⟨APRNN⟩.

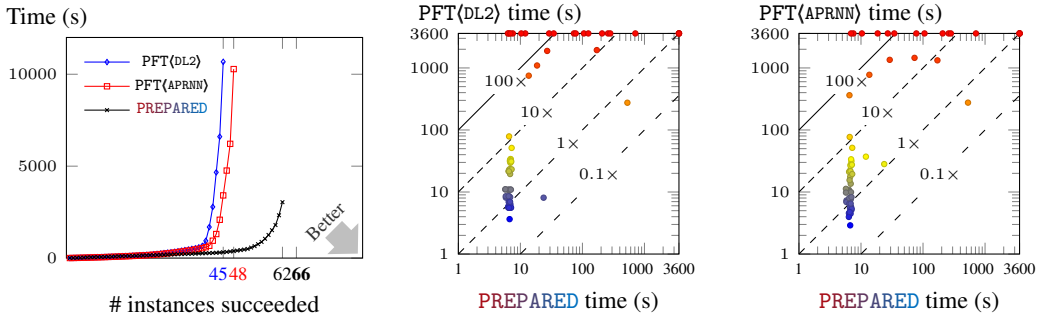

(b) **Left**: cactus plot of runtime for 66 all-properties editing problems, with 3600 seconds time limit for each problem. PREPARED succeeded on 62 problems, while PFT⟨DL2⟩ succeeded on 45, and PFT⟨APRNN⟩ succeeded 48. **Middle**: speed up for PREPARED vs PFT⟨DL2⟩. **Right**: speed up for PREPARED vs PFT⟨APRNN⟩.

Figure 4: Results of **(a)** single-property and **(b)** all-properties editing problems from VNN-COMP'22.

**Theorem 3.14.** *Definition 3.13 is a Parametric Linear Relaxation for the Affine layer with variable input bounds $[\dot{\mathbf{x}}, \ddot{\mathbf{x}}]$ that captures the exact upper and lower bounds.*

### 3.6 Parametric Linear Relaxation for Tanh, Sigmoid and ELU layers

Our approach can also handle other activation layers like Tanh, Sigmoid and ELU. Figure 3 illustrates their Parametric Linear Relaxations with detailed description deferred to Appendix C.

### 3.7 On scalability

As described in Appendix B, our approach can restrict the edits to only the last $k$ layers of the DNN $\mathcal{N}$, freezing the parameters of the first $L - k$ layers. Consequently, the resulting Parametric Linear Relaxation is a linear formula whose size is polynomial in the size of *only the last $k$ editable layers $\mathcal{N}^{(k:L)}$* instead of the entire DNN $\mathcal{N}$. This flexibility enables our approach to scale to large DNNs and BERT transformers, as demonstrated in Section 4.

## 4 Experimental evaluation

We have implemented PREPARED in PyTorch [32] and use Gurobi [17] as the LP solver. We demonstrate the effectiveness and efficiency of PREPARED on different tasks that use a wide-range of DNN architectures and properties. All experiments were run on a machine with Dual Intel Xeon Silver 4216 Processor 16-Core 2.1GHz with 384 GB of memory, SSD and RTX-A6000 with 48 GB of GPU memory running Ubuntu 20.04. Additional details about these experiments are in Appendix D.

**Baselines.** Because there were no efficient prior approaches for provable editing, we implemented *provable fine-tuning* PFT⟨·⟩ baselines that combine prior (non-provable) DNN editing approaches with

a verifier in the loop: the editing stops when the verifier confirms that the DNN satisfies the property (Appendix D.1). We consider the state-of-the-art DNN editing approaches DL2 [10], APRNN [42], SABR [28] and STAPS [24] as introduced in Section 2. We use PFT⟨DL2⟩ to denote the provable fine-tuning baseline instantiated with DL2. We use verifiers $\alpha,\beta$-CROWN [46, 44], MN-BaB [9], or DeepT [4], depending on the task. To the best of our knowledge, ours is the *first* to provide a comprehensive evaluation of such verifier-in-the-loop baselines for provable editing.

## 4.1 Provable editing on VNN competition benchmarks

**Setup.** We compare PREPARED against PFT⟨DL2⟩ and PFT⟨APRNN⟩ on VNN-COMP′22 benchmarks [29]. The VNN-COMP′22 benchmarks consist of DNNs along with one or more of their associated safety properties. For verification, the task would be to determine whether a DNN satisfies each of its properties. In the context of provable editing, we use these benchmarks in two scenarios: **i) Single-property editing**: Given a DNN and a property it violates, edit it to satisfy this single property with a time limit of 600 seconds. There are 423 such DNN-property instances. **ii) All-properties editing**: Given a DNN that violates at least one of its associated properties, edit it to satisfy the conjunction of all (satisfied or violated) properties associated with it with a time limit of 3600 seconds. There are 66 such DNN-property instances.

PFT⟨SABR⟩ and PFT⟨STAPS⟩ are not used because they do not handle all properties and DNN architectures in this experiment. In particular, SABR and STAPS have not discussed how to handle general logical formula with disjunctions, and their current implementations are designed for local robustness training. PFT⟨SABR⟩ and PFT⟨STAPS⟩ are used for local robustness editing in Section 4.2.

**Metrics.** The **effectiveness** is measured using the number of provably edited instances, and the **efficiency** is measured using the runtime. We were unable to determine the impact on the predictive performance (accuracy), because VNN-COMP does not come with such evaluation metrics and data.

**Results.** As shown in Figure 4, PREPARED *is the best provable editing approach*, significantly outperforming PFT⟨DL2⟩ and PFT⟨APRNN⟩ in terms of both *effectiveness* and *efficiency*. As shown in the cactus plots (**Left** in Figures 4(a) and 4(b)), PREPARED can provably edit more instances in less time. As shown in the speed-up plots (**Middle** and **Right** in Figures 4(a) and 4(b)), PREPARED completes most problems in 10 seconds, taking a maximum of 45 seconds for single-property editing; it achieves from 10× to more than 100× speed-up over PFT⟨DL2⟩ and PFT⟨APRNN⟩. Specifically, PREPARED succeeds on all 423 single-property editing problems, while PFT⟨DL2⟩ and PFT⟨APRNN⟩ succeed on 185 and 117 problems, respectively. PREPARED succeeds on 62 out of 66 multi-property editing problems, while PFT⟨DL2⟩ and PFT⟨APRNN⟩ only succeed on 45 and 48 problems, respectively.

## 4.2 Local robustness editing for image-recognition DNNs

**Definition 4.1.** Consider a classifier DNN $\mathcal{N} : \mathbb{R}^m \rightarrow \mathbb{R}^n$ and a dataset $\mathcal{D}$ of input-label pairs $(\mathbf{x}, l) \in \mathbb{R}^m \times \mathbb{Z}$. The DNN $\mathcal{N}$ *correctly classifies* an input-label pair $(\mathbf{x}, l)$ iff $\arg\max \mathcal{N}(\mathbf{x}) = l$, and the *standard accuracy* of a DNN on a dataset $\mathcal{D}$ is the percentage of correctly classified input-label pairs in $\mathcal{D}$. Given a perturbation $\varepsilon \in \mathbb{R}$, the DNN $\mathcal{N}$ is $L^\infty$-*locally-robust* on an input-label pair $(\mathbf{x}, l)$ iff $\mathcal{N}$ correctly classifies all perturbed inputs $\mathbf{x}'$ in the $L^\infty$ box centered at $\mathbf{x}$ with perturbation $\varepsilon$, viz., $\forall \mathbf{x}' \in \mathbb{R}^m . \|\mathbf{x}' - \mathbf{x}\|_\infty \leq \varepsilon \implies \arg\max \mathcal{N}(\mathbf{x}') = l$. The *certified accuracy* of a DNN on a dataset $\mathcal{D}$ is the percentage of $L^\infty$-locally-robust input-label pairs in $\mathcal{D}$.

**Setup.** We compare PREPARED against PFT⟨DL2⟩, PFT⟨SABR⟩, and APRNN on $L^\infty$-local-robustness editing for image-recognition DNNs. We edit CNN7 DNNs for CIFAR10 [21] and TINYIMA-GENET [33], trained in prior work [28], to be locally-robust for images in the edit set. The CIFAR10 DNN has 17.2M parameters, 79.24% standard accuracy and 75.77% certified accuracy ($\varepsilon = 0.5/255$). The TinyImageNet DNN has 51.9M parameters, 28.85% standard accuracy and 20.46% certified accuracy ($\varepsilon = 1/255$). For CIFAR10 and TINYIMAGENET, the edit sets consist of 50 misclassified fog-corrupted images from CIFAR10-C [18] and TINYIMAGENET-C [18], respectively. The generalization sets consist of 200 variant images with different corruption-levels, four variants per image in the edit set. We use MN-BaB [9] to compute the certified accuracy.

**Metrics. Efficacy** is measured using the certified accuracy (Definition 4.1) on the edit set. The original certified accuracy on the edit set is 0%. Efficacy is the most important metric: *a provable edit must guarantee 100% efficacy.* The **standard accuracy** and **certified accuracy** on the full

Table 1: Local robustness editing for image-recognition DNNs. Comparison of efficacy (Effic.), standard (Acc.), certified (Cert.) and generalization (Gen.) accuracy. Rows with <100% efficacy are shaded. Best results are **highlighted**.

| Method | CIFAR10 with $\varepsilon={0.5}/255$ | | | | | TINYIMAGENET with $\varepsilon={1}/255$ | | | | |
|---|---|---|---|---|---|---|---|---|---|---|
| | Effic. | Acc. | Cert. | Gen. | Time | Effic. | Acc. | Cert. | Gen. | Time |
| PFT⟨DL2⟩ | 50/50 | 73.59% | 70.42% | 197/200 | 4780s | 47/50 | 22.83% | 15.06% | 190/200 | 14400s* |
| PFT⟨SABR⟩ | 50/50 | 57.10% | 52.72% | **200/200** | 38s | 50/50 | 20.24% | 13.38% | **192/200** | 230s |
| PFT⟨STAPS⟩ | 50/50 | 57.38% | 54.26% | 197/200 | **15s** | 50/50 | 20.84% | 13.14% | 191/200 | **112s** |
| APRNN | 5/50 | 78.96% | 75.14% | 113/200 | 20s | 12/50 | 28.75% | 18.61% | 115/200 | 417s |
| PREPARED | 50/50 | **75.25%** | **71.45%** | 189/200 | 88s | 50/50 | **28.16%** | **16.80%** | 149/200 | 663s |

\* Timeout in four hours.

Table 2: Local robustness editing for sentiment-classification transformers. Comparison of the lower bound of efficacy before (Og. Effic.) and after edit (Effic.), as well as the stand accuracy (Acc.). Rows with <100% efficacy are shaded. Best results are **highlighted**.

| Method | $\varepsilon = 1e\text{-}4$ | | | | $\varepsilon = 5e\text{-}4$ | | | |
|---|---|---|---|---|---|---|---|---|
| | Og. Effic. | Effic. | Acc. | Time | Og. Effic. | Effic. | Acc. | Time |
| PFT⟨DL2⟩ | | 52/66 | 82.97% | 12108s | | 23/26 | 79.67% | 5610s |
| APRNN | 60/66 | 61/66 | 83.47% | 6s | 25/26 | 23/26 | 81.32% | 5s |
| PREPARED | | **66/66** | **83.52%** | 981s | | **26/26** | **82.42%** | **326s** |

test set are also important metrics: a good provable edit should have high accuracy. However, those accuracy metrics are relevant only if efficacy is 100%. The **generalization** accuracy on the generalization set measures how well the edit generalizes to inputs that are similar to the edit set. However, the generalization accuracy is relevant only if the efficacy is 100% and the standard and certified accuracies are good; that is, the predictive power of the edited DNN should not be sacrificed for better generalization. The **runtime** metric measures the time taken to edit the DNN.

**Results.**  As shown in Table 1, PREPARED *is overall the best approach.* PREPARED achieves the best standard and certified accuracy, good generalization and short runtime. In particular, PFT⟨SABR⟩ and PFT⟨STAPS⟩ have significantly lower standard and certified accuracy; PFT⟨DL2⟩ takes significantly longer time, and was unable to achieve 100% efficacy in four hours in the TINYIMAGENET experiment; APRNN was unable to achieve 100% efficacy.

### 4.3 Local robustness editing for sentiment classification BERT transformers

**Setup.**  We compare PREPARED against PFT⟨DL2⟩ and APRNN on provable $L^\infty$ local-robustness editing for BERT sentiment-classification transformers. We conduct this experiment on the Stanford Sentiment Treebank (SST) [40] DNNs. We use DeepT [4] as the verifier in this experiment. We take the "wider" 12-layer BERT transformer used in the DeepT paper [4]. The standard accuracy of this network is 84.07%. Because DeepT cannot handle all sentences in the SST dataset with arbitrary $\varepsilon$, we construct two editing sets from the SST validation set: all 66 verifiable sentences with $\varepsilon = 1e\text{-}4$, where 60 of them are certified to be locally-robust; all 26 verifiable sentences with $\varepsilon = 5e\text{-}4$, where 24 of them are certified to be locally-robust. PFT⟨SABR⟩ and PFT⟨STAPS⟩ are not compared in this experiment because SABR and STAPS do not handle the BERT transformer architecture.

**Metrics.  Efficacy** is measured using the certified accuracy (Definition 4.1) on the edit set. Because DeepT is incomplete, we can only compute a lower bound of the efficacy. Efficacy is the most important metric: *a provable edit must guarantee 100% efficacy.* The original efficacy (Og. Effic.) is also presented in Table 2. The **standard accuracy** on the full test set is also an important metric: a good provable edit should have high accuracy. However, the standard accuracy is relevant only if efficacy is 100%. The **runtime** metric measures the time taken to edit the DNN.

**Results.**  As seen in Table 2, PREPARED is the only approach that achieves 100% efficacy in this task. PREPARED also achieves good accuracy and short runtime. Both PFT⟨DL2⟩ and APRNN were unable to achieve 100% efficacy, and decreased the efficacy in most experiments.

Table 3: Global physics property repair for geodynamics DNN. Comparison of relative error (Rel. Err.), continuity (Cont. Err.) and boundary-condition (BC Err.) errors. Errors on the test set (Ref.) are shaded. Best results are **highlighted**.

| Method | Rel. Err. | Cont. Err. | BC Err. | Time |
|:------:|:---------:|:----------:|:-------:|:----:|
| Ref. | — | $1.42 \times 10^{-12}$ | $1.74 \times 10^{-12}$ | — |
| DL2 | 1.59% | $1.58 \times 10^{-4}$ | 5.08 | 14 674s |
| GD | **0.26%** | $1.59 \times 10^{-5}$ | $5.86 \times 10^{-1}$ | **395s** |
| GD⟨APRNN⟩ | 0.50% | $5.57 \times 10^{-7}$ | $4.96 \times 10^{-5}$ | 570s |
| GD⟨PREPARED⟩ | 0.50% | $\mathbf{1.42 \times 10^{-12}}$ | $\mathbf{2.29 \times 10^{-7}}$ | 684s |

## 4.4 Provable training for physics-plausible DNNs

**Numerical model.** We consider a classic model of a buoyancy-driven mantle flow with a central circular plume [13], which is an incompressible flow with variable viscosity. The state of the system is a vector field $\mathbf{F} \overset{\text{def}}{=} (u, v, \eta, \rho)$ of velocity field $\mathbf{U} \overset{\text{def}}{=} (u, v)$, viscosity $\eta$ and density $\rho$, over space $(x, y) \in \Omega$ and time $t \in \mathcal{T}$. We consider the following two physics constraints: **i) Conservation of mass** $\nabla \cdot \mathbf{U} = 0$ from the **continuity** equation; **ii) Dirichlet boundary condition** $\mathbf{U} \cdot \mathbf{n} = 0$ on the boundary $\partial \Omega$, where $\mathbf{n}$ is the outward normal of $\partial \Omega$.

**Setup.** We compare GD⟨PREPARED⟩, a combination of gradient-descent (GD) and PREPARED, against DL2, GD and GD⟨APRNN⟩ on the task of provably training a DNN to model the geodynamics process and satisfy physics constraints. We discretize the space into a 31×31 grid and implement the numerical model to generate data for 150 time steps. We evenly split the data into training, validation and test sets. We train a ReLU ResNet with one single residual block of four conv2d layers with 32 channels and 3×3 kernel size. For DL2, we add the physics constraints as an unsupervised-learning regularization to the supervised training. For GD⟨APRNN⟩ and GD⟨PREPARED⟩, we first use vanilla gradient-descent (GD) to train a base model, then edit it with the constraints. Specifically, GD⟨APRNN⟩ edits all training points, and GD⟨PREPARED⟩ edits the convex-hull of all training points to satisfy the constraints.

**Metrics.** The **relative error** on the test set, the constraint-satisfaction errors, viz., **continuity error**, for the conservation of mass, and **boundary-condition error**, and the **runtime** are the metrics.

**Results.** As shown in Table 3, GD⟨PREPARED⟩ *is overall the best approach*. GD⟨PREPARED⟩ has good relative error, and its constraint-satisfaction error is significantly better than other approaches. Notably, its continuity error is at the same magnitude as the reference data, close to the theoretical value, zero, and is $10^5$ to $10^8$ times better than other approaches. GD achieves the best relative error and the shortest runtime. DL2 has the worst errors and takes significantly longer amount of time.

## 5 Conclusion

We have presented PREPARED, the first efficient approach for provable editing of DNNs that runs in polynomial time. We presented novel methods for constructing tight parametric bounds for the DNN output using Parametric Linear Relaxation, enabling PREPARED to use an LP solver to find an edit. To demonstrate its effectiveness and efficiency, PREPARED was used to provably edit DNNs and properties from the VNN-COMP′22 benchmark, provably edit CIFAR10 and TINYIMAGENET image-recognition DNNs, and BERT sentiment-classification DNNs for local robustness, and provably train a geodynamics DNN to satisfy physics constraints.

**Societal impact.** PREPARED represents a step towards the goal of ensuring safety, trustworthiness, and reliability of DNNs, which is crucial given their use in autonomous safety-critical systems. However, being a general technique for editing DNNs, it could be used to remove or add malicious behavior such as bias, backdoor attacks, etc.

**Limitations.** PREPARED requires as input a property that we wish the DNN to satisfy. These properties would need to formally encode the notions of safety, reliability, or trustworthiness, which may not always be possible or easy to do. e.g., defining safety properties for LLM-based chatbots. However, the machine learning and formal methods communities are making progress towards this goal; e.g., developing proxy specifications for complex properties that are easier to define and learning safety specifications from data [8].

## Acknowledgments and Disclosure of Funding

We would like to thank the anonymous reviewers for their feedback and suggestions, which have greatly improved the quality of the paper. This work is supported in part by NSF grant CCF-2048123 and DOE Award DE-SC0022285.

## Footnotes

[1]PREPARED stands for PRovable Editing using PArametric linear RElaxation of Deep neural networks.

## References

[1] Aws Albarghouthi. Introduction to neural network verification. *Found. Trends Program. Lang.*, 7(1-2): 1–157, 2021. doi: 10.1561/2500000051. URL `https://doi.org/10.1561/2500000051`.

[2] Mislav Balunovic and Martin Vechev. Adversarial training and provable defenses: Bridging the gap. In *International Conference on Learning Representations*, 2020. URL `https://openreview.net/forum?id=SJxSDxrKDr`.

[3] Gregory Bonaert, Dimitar I. Dimitrov, Maximilian Baader, and Martin T. Vechev. Fast and precise certification of transformers. `https://github.com/eth-sri/DeepT/tree/16ffe4075f1f8a7c87fa2a187d8c46cfd51e07bf`, 2021.

[4] Gregory Bonaert, Dimitar I. Dimitrov, Maximilian Baader, and Martin T. Vechev. Fast and precise certification of transformers. In Stephen N. Freund and Eran Yahav, editors, *PLDI '21: 42nd ACM SIGPLAN International Conference on Programming Language Design and Implementation, Virtual Event, Canada, June 20-25, 2021*, pages 466–481. ACM, 2021. doi: 10.1145/3453483.3454056. URL `https://doi.org/10.1145/3453483.3454056`.

[5] Christopher Brix, Mark Niklas Müller, Stanley Bak, Taylor T. Johnson, and Changliu Liu. First three years of the international verification of neural networks competition (vnn-comp), 2023.

[6] George E. Collins. Quantifier elimination for real closed fields by cylindrical algebraic decompostion. In H. Brakhage, editor, *Automata Theory and Formal Languages*, pages 134–183, Berlin, Heidelberg, 1975. Springer Berlin Heidelberg. ISBN 978-3-540-37923-2.

[7] George E. Collins and Hoon Hong. Partial cylindrical algebraic decomposition for quantifier elimination. *Journal of Symbolic Computation*, 12(3):299–328, 1991. ISSN 0747-7171. doi: https://doi.org/10.1016/S0747-7171(08)80152-6. URL `https://www.sciencedirect.com/science/article/pii/S0747717108801526`.

[8] David Dalrymple, Joar Skalse, Yoshua Bengio, Stuart Russell, Max Tegmark, Sanjit Seshia, Steve Omohundro, Christian Szegedy, Ben Goldhaber, Nora Ammann, Alessandro Abate, Joe Halpern, Clark Barret, Ding Zhao, Tan Zhi-Xuan, Jeannette Wing, and Joshua Tenenbaum. Towards guaranteed safe ai: A framework for ensuring robust and reliable AI systems. *arXiv preprint arXiv:2405.06624*, 2024.

[9] Claudio Ferrari, Mark Niklas Müller, Nikola Jovanović, and Martin Vechev. Complete verification via multi-neuron relaxation guided branch-and-bound. In *International Conference on Learning Representations*, 2022.

[10] Marc Fischer, Mislav Balunovic, Dana Drachsler-Cohen, Timon Gehr, Ce Zhang, and Martin T. Vechev. DL2: training and querying neural networks with logic. In *Proceedings of the 36th International Conference on Machine Learning (ICML)*, volume 97 of *Proceedings of Machine Learning Research*. PMLR, 2019.

[11] Marc Fischer, Mislav Balunovic, Dana Drachsler-Cohen, Timon Gehr, Ce Zhang, and Martin T. Vechev. DL2: Training and querying neural networks with logic. `https://github.com/eth-sri/dl2/tree/05e2d3fb4e2b00ef5968bf28d26219750a580908`, 2023.

[12] Feisi Fu and Wenchao Li. Sound and complete neural network repair with minimality and locality guarantees. In *10th International Conference on Learning Representations, ICLR 2022, Lisbon, Portugal, Oct 27-28, 2022*. OpenReview.net, 2022. URL `https://arxiv.org/abs/2110.07682`.

[13] Taras Gerya. *Introduction to numerical geodynamic modelling*. Cambridge University Press, 2019.

[14] Ben Goldberger, Guy Katz, Yossi Adi, and Joseph Keshet. Minimal modifications of deep neural networks using verification. In Elvira Albert and Laura Kovács, editors, *LPAR 2020: 23rd International Conference on Logic for Programming, Artificial Intelligence and Reasoning, Alicante, Spain, May 22-27, 2020*, volume 73 of *EPiC Series in Computing*, pages 260–278. EasyChair, 2020. doi: 10.29007/699q. URL `https://doi.org/10.29007/699q`.

[15] Sven Gowal, Krishnamurthy Dvijotham, Robert Stanforth, Rudy Bunel, Chongli Qin, Jonathan Uesato, Relja Arandjelovic, Timothy Mann, and Pushmeet Kohli. On the effectiveness of interval bound propagation for training verifiably robust models, 2019.

[16] Kshitij Goyal, Sebastijan Dumancic, and Hendrik Blockeel. Deepsade: Learning neural networks that guarantee domain constraint satisfaction. *Proceedings of the AAAI Conference on Artificial Intelligence*, 38 (11):12199–12207, Mar. 2024. doi: 10.1609/aaai.v38i11.29109. URL `https://ojs.aaai.org/index.php/AAAI/article/view/29109`.

[17] Gurobi Optimization, LLC. Gurobi Optimizer Reference Manual, 2022. URL `https://www.gurobi.com`.

[18] Dan Hendrycks and Thomas G. Dietterich. Benchmarking neural network robustness to common corruptions and perturbations. In *7th International Conference on Learning Representations, ICLR 2019, New Orleans, LA, USA, May 6-9, 2019*. OpenReview.net, 2019. URL `https://openreview.net/forum?id=HJz6tiCqYm`.

[19] Nick Hoernle, Rafael Michael Karampatsis, Vaishak Belle, and Kobi Gal. Multiplexnet: Towards fully satisfied logical constraints in neural networks. *Proceedings of the AAAI Conference on Artificial Intelligence*, 36(5):5700–5709, Jun. 2022. doi: 10.1609/aaai.v36i5.20512. URL `https://ojs.aaai.org/index.php/AAAI/article/view/20512`.

[20] Zhiting Hu, Xuezhe Ma, Zhengzhong Liu, Eduard Hovy, and Eric Xing. Harnessing deep neural networks with logic rules. In *Proceedings of the 54th Annual Meeting of the Association for Computational Linguistics (Volume 1: Long Papers)*, pages 2410–2420, Berlin, Germany, August 2016. Association for Computational Linguistics. doi: 10.18653/v1/P16-1228. URL `https://aclanthology.org/P16-1228`.

[21] Alex Krizhevsky, Vinod Nair, and Geoffrey Hinton. Cifar-10 (canadian institute for advanced research). URL `http://www.cs.toronto.edu/~kriz/cifar.html`.

[22] Zenan Li, Zehua Liu, Yuan Yao, Jingwei Xu, Taolue Chen, Xiaoxing Ma, and Jian L\"{u}. Learning with logical constraints but without shortcut satisfaction. In *The Eleventh International Conference on Learning Representations*, 2023. URL `https://openreview.net/forum?id=M2unceRvqhh`.

[23] Dmitry Malioutov and Kuldeep S Meel. Mlic: A maxsat-based framework for learning interpretable classification rules. In *International Conference on Principles and Practice of Constraint Programming*, pages 312–327. Springer, 2018.

[24] Yuhao Mao, Mark Niklas Mueller, Marc Fischer, and Martin Vechev. Connecting certified and adversarial training. In *Thirty-seventh Conference on Neural Information Processing Systems*, 2023. URL `https://openreview.net/forum?id=T2lM4ohRwb`.

[25] Yuhao Mao, Mark Niklas Mueller, Marc Fischer, and Martin Vechev. TAPS: Connecting certified and adversarial training. `https://github.com/eth-sri/TAPS/tree/6b5c5d9d3453c482f56f00e6f57b99b07b11ca4f`, 2023.

[26] Matthew Mirman, Timon Gehr, and Martin T. Vechev. Differentiable abstract interpretation for provably robust neural networks. In Jennifer G. Dy and Andreas Krause, editors, *Proceedings of the 35th International Conference on Machine Learning, ICML 2018, Stockholmsmässan, Stockholm, Sweden, July 10-15, 2018*, volume 80 of *Proceedings of Machine Learning Research*, pages 3575–3583. PMLR, 2018. URL `http://proceedings.mlr.press/v80/mirman18b.html`.

[27] Mark Niklas Mueller, Franziska Eckert, Marc Fischer, and Martin Vechev. SABR: Small adversarial bounding boxes. `https://github.com/eth-sri/sabr`, 2023.

[28] Mark Niklas Mueller, Franziska Eckert, Marc Fischer, and Martin Vechev. Certified training: Small boxes are all you need. In *The Eleventh International Conference on Learning Representations*, 2023. URL `https://openreview.net/forum?id=7oFuxtJtUMH`.

[29] Mark Niklas Müller, Christopher Brix, Stanley Bak, Changliu Liu, and Taylor T. Johnson. The third international verification of neural networks competition (vnn-comp 2022): Summary and results, 2022. URL `https://arxiv.org/abs/2212.10376`.

[30] Yatin Nandwani, Abhishek Pathak, Mausam, and Parag Singla. A primal dual formulation for deep learning with constraints. In H. Wallach, H. Larochelle, A. Beygelzimer, F. d'Alché-Buc, E. Fox, and R. Garnett, editors, *Advances in Neural Information Processing Systems*, volume 32. Curran Associates, Inc., 2019. URL `https://proceedings.neurips.cc/paper_files/paper/2019/file/cf708fc1decf0337aded484f8f4519ae-Paper.pdf`.

[31] Alessandro De Palma, Rudy Bunel, Krishnamurthy Dvijotham, M. Pawan Kumar, and Robert Stanforth. Ibp regularization for verified adversarial robustness via branch-and-bound, 2023.

[32] Adam Paszke, Sam Gross, Francisco Massa, Adam Lerer, James Bradbury, Gregory Chanan, Trevor Killeen, Zeming Lin, Natalia Gimelshein, Luca Antiga, Alban Desmaison, Andreas Köpf, Edward Z. Yang, Zach DeVito, Martin Raison, Alykhan Tejani, Sasank Chilamkurthy, Benoit Steiner, Lu Fang, Junjie Bai, and Soumith Chintala. Pytorch: An imperative style, high-performance deep learning library. *CoRR*, abs/1912.01703, 2019. URL `http://arxiv.org/abs/1912.01703`.

[33] Hadi Pouransari and Saman Ghili. Tiny imagenet visual recognition challenge. *CS 231N*, 7, 2014.

[34] Hadi Salman, Greg Yang, Huan Zhang, Cho-Jui Hsieh, and Pengchuan Zhang. A convex relaxation barrier to tight robustness verification of neural networks. In Hanna M. Wallach, Hugo Larochelle, Alina Beygelzimer, Florence d'Alché-Buc, Emily B. Fox, and Roman Garnett, editors, *Advances in Neural Information Processing Systems 32: Annual Conference on Neural Information Processing Systems 2019, NeurIPS 2019, December 8-14, 2019, Vancouver, BC, Canada*, pages 9832–9842, 2019. URL `https://proceedings.neurips.cc/paper/2019/hash/246a3c5544feb054f3ea718f61adfa16-Abstract.html`.

[35] Sanjit A. Seshia, Dorsa Sadigh, and S. Shankar Sastry. Toward verified artificial intelligence. *Commun. ACM*, 65(7):46–55, 2022. doi: 10.1145/3503914. URL `https://doi.org/10.1145/3503914`.

[36] Zhouxing Shi, Yihan Wang, Huan Zhang, Jinfeng Yi, and Cho-Jui Hsieh. Fast certified robust training with short warmup. In *ICML 2021 Workshop on Adversarial Machine Learning*, 2021. URL `https://openreview.net/forum?id=_jUobmvki51`.

[37] Zhouxing Shi, Qirui Jin, J Zico Kolter, Suman Jana, Cho-Jui Hsieh, and Huan Zhang. Formal verification for neural networks with general nonlinearities via branch-and-bound. 2023.

[38] Gagandeep Singh, Timon Gehr, Matthew Mirman, Markus Püschel, and Martin T. Vechev. Fast and effective robustness certification. In Samy Bengio, Hanna M. Wallach, Hugo Larochelle, Kristen Grauman, Nicolò Cesa-Bianchi, and Roman Garnett, editors, *Advances in Neural Information Processing Systems 31: Annual Conference on Neural Information Processing Systems 2018, NeurIPS 2018, December 3-8, 2018, Montréal, Canada*, pages 10825–10836, 2018. URL `https://proceedings.neurips.cc/paper/2018/hash/f2f446980d8e971ef3da97af089481c3-Abstract.html`.

[39] Gagandeep Singh, Timon Gehr, Markus Püschel, and Martin T. Vechev. An abstract domain for certifying neural networks. *Proc. ACM Program. Lang.*, 3(POPL):41:1–41:30, 2019. doi: 10.1145/3290354. URL `https://doi.org/10.1145/3290354`.

[40] Richard Socher, Alex Perelygin, Jean Wu, Jason Chuang, Christopher D Manning, Andrew Y Ng, and Christopher Potts. Recursive deep models for semantic compositionality over a sentiment treebank. In *Proceedings of the 2013 conference on empirical methods in natural language processing*, pages 1631–1642, 2013.

[41] Matthew Sotoudeh and Aditya Thakur. Prdnn. `https://github.com/95616ARG/PRDNN`, 2021.

[42] Zhe Tao, Stephanie Nawas, Jacqueline Mitchell, and Aditya V. Thakur. Architecture-preserving provable repair of deep neural networks. *Proc. ACM Program. Lang.*, 7(PLDI), jun 2023. doi: 10.1145/3591238. URL `https://doi.org/10.1145/3591238`.

[43] Christian Tjandraatmadja, Ross Anderson, Joey Huchette, Will Ma, Krunal Patel, and Juan Pablo Vielma. The convex relaxation barrier, revisited: Tightened single-neuron relaxations for neural network verification. In Hugo Larochelle, Marc'Aurelio Ranzato, Raia Hadsell, Maria-Florina Balcan, and Hsuan-Tien Lin, editors, *Advances in Neural Information Processing Systems 33: Annual Conference on Neural Information Processing Systems 2020, NeurIPS 2020, December 6-12, 2020, virtual*, 2020. URL `https://proceedings.neurips.cc/paper/2020/hash/f6c2a0c4b566bc99d596e58638e342b0-Abstract.html`.

[44] Shiqi Wang, Huan Zhang, Kaidi Xu, Xue Lin, Suman Jana, Cho-Jui Hsieh, and J. Zico Kolter. Beta-crown: Efficient bound propagation with per-neuron split constraints for neural network robustness verification. In Marc'Aurelio Ranzato, Alina Beygelzimer, Yann N. Dauphin, Percy Liang, and Jennifer Wortman Vaughan, editors, *Advances in Neural Information Processing Systems 34: Annual Conference on Neural Information Processing Systems 2021, NeurIPS 2021, December 6-14, 2021, virtual*, pages 29909–29921, 2021. URL `https://proceedings.neurips.cc/paper/2021/hash/fac7fead96dafceaf80c1daffeae82a4-Abstract.html`.

[45] Jingyi Xu, Zilu Zhang, Tal Friedman, Yitao Liang, and Guy Van den Broeck. A semantic loss function for deep learning with symbolic knowledge. In Jennifer Dy and Andreas Krause, editors, *Proceedings of the 35th International Conference on Machine Learning*, volume 80 of *Proceedings of Machine Learning Research*, pages 5502–5511. PMLR, 10–15 Jul 2018. URL `https://proceedings.mlr.press/v80/xu18h.html`.

[46] Kaidi Xu, Huan Zhang, Shiqi Wang, Yihan Wang, Suman Jana, Xue Lin, and Cho-Jui Hsieh. Fast and complete: Enabling complete neural network verification with rapid and massively parallel incomplete verifiers. In *9th International Conference on Learning Representations, ICLR 2021, Virtual Event, Austria, May 3-7, 2021*. OpenReview.net, 2021. URL `https://openreview.net/forum?id=nVZtXBI6LNn`.

[47] Ziwei Xu, Yogesh Rawat, Yongkang Wong, Mohan S Kankanhalli, and Mubarak Shah. Don't pour cereal into coffee: Differentiable temporal logic for temporal action segmentation. In S. Koyejo, S. Mohamed, A. Agarwal, D. Belgrave, K. Cho, and A. Oh, editors, *Advances in Neural Information Processing Systems*, volume 35, pages 14890–14903. Curran Associates, Inc., 2022. URL `https://proceedings.neurips.cc/paper_files/paper/2022/file/5f96a21345c138da929e99871fda138e-Paper-Conference.pdf`.

[48] Huan Zhang, Tsui-Wei Weng, Pin-Yu Chen, Cho-Jui Hsieh, and Luca Daniel. Efficient neural network robustness certification with general activation functions. In Samy Bengio, Hanna M. Wallach, Hugo Larochelle, Kristen Grauman, Nicolò Cesa-Bianchi, and Roman Garnett, editors, *Advances in Neural Information Processing Systems 31: Annual Conference on Neural Information Processing Systems 2018, NeurIPS 2018, December 3-8, 2018, Montréal, Canada*, pages 4944–4953, 2018. URL `https://proceedings.neurips.cc/paper/2018/hash/d04863f100d59b3eb688a11f95b0ae60-Abstract.html`.

[49] Huan Zhang, Tsui-Wei Weng, Pin-Yu Chen, Cho-Jui Hsieh, and Luca Daniel. Efficient neural network robustness certification with general activation functions. *Advances in neural information processing systems*, 31, 2018.

[50] Huan Zhang, Zhouxing Shi, Kaidi Xu, Yihan Wang, Shiqi Wang, Linyi Li, Jinqi (Kathryn) Chen, and Zhuolin Yang. auto_LiRPA: An automatic linear relaxation based perturbation analysis library for neural networks and general computational graphs. `https://github.com/Verified-Intelligence/auto_LiRPA`, 2023.

# A  Proofs

**Theorem 3.4.** *Given a provable interval editing problem (Definition 3.2) for DNN $\mathcal{N}$ and parameters $\theta$ with input bounds $[\underline{\mathbf{x}}, \overline{\mathbf{x}}]$ and output bounds $[\underline{\mathbf{y}}, \overline{\mathbf{y}}]$, let $\overline{\varphi}_{\mathcal{N}}(\dot{\underline{\mathbf{y}}}, \dot{\overline{\mathbf{y}}}, \underline{\mathbf{x}}, \overline{\mathbf{x}}; \dot{\theta})$ be a Parametric Linear Relaxation for $\mathcal{N}$. Then the following linear program can be solved in polynomial time in the size of the DNN $\mathcal{N}$, and whose solution is a solution to the provable interval editing problem:*

$$\min \|\dot{\theta} - \theta\| \quad s.t. \quad \overline{\varphi}_{\mathcal{N}}(\dot{\underline{\mathbf{y}}}, \dot{\overline{\mathbf{y}}}, \underline{\mathbf{x}}, \overline{\mathbf{x}}; \dot{\theta}) \wedge (\underline{\mathbf{y}} \leq \dot{\underline{\mathbf{y}}} \wedge \dot{\overline{\mathbf{y}}} \leq \overline{\mathbf{y}}) \tag{4}$$

*Proof.* Equation 4 is a linear program that can be solved in polynomial in the size of the DNN $\mathcal{N}$ because by Definition 3.3, $\overline{\varphi}_{\mathcal{N}}(\dot{\underline{\mathbf{y}}}, \dot{\overline{\mathbf{y}}}, \underline{\mathbf{x}}, \overline{\mathbf{x}}; \dot{\theta})$ is a poly-size linear formula whose size is polynomial in the size of the DNN $\mathcal{N}$, i.e., the number of parameters, layers, and the input and output dimensions of each layer.

Equation 4 is a solution to the provable interval editing problem (Definition 3.2) because by Definition 3.3, $\overline{\varphi}_{\mathcal{N}}(\dot{\underline{\mathbf{y}}}, \dot{\overline{\mathbf{y}}}, \underline{\mathbf{x}}, \overline{\mathbf{x}}; \dot{\theta})$ implies $\forall \mathbf{x} \in [\underline{\mathbf{x}}, \overline{\mathbf{x}}]. \; \mathcal{N}(\mathbf{x}; \dot{\theta}) \in [\dot{\underline{\mathbf{y}}}, \dot{\overline{\mathbf{y}}}]$. $\qquad\square$

**Theorem 3.6.** *Definition 3.5 is a Parametric Linear Relaxation for the DNN $\mathcal{N}$.*

*Proof.* Recall Definition 3.5:

$$\overline{\varphi}_{\mathcal{N}}(\dot{\underline{\mathbf{y}}}, \dot{\overline{\mathbf{y}}}, \underline{\mathbf{x}}, \overline{\mathbf{x}}; \dot{\theta}) \stackrel{\text{def}}{=} \bigwedge_{0 \leq \ell < L} \overline{\varphi}_{\mathcal{N}^{(\ell)}}(\dot{\underline{\mathbf{x}}}^{(\ell+1)}, \dot{\overline{\mathbf{x}}}^{(\ell+1)}, \dot{\underline{\mathbf{x}}}^{(\ell)}, \dot{\overline{\mathbf{x}}}^{(\ell)}; \dot{\theta}^{(\ell)})$$

where $[\dot{\underline{\mathbf{y}}}, \dot{\overline{\mathbf{y}}}] \stackrel{\text{def}}{=} [\dot{\underline{\mathbf{x}}}^{(L)}, \dot{\overline{\mathbf{x}}}^{(L)}]$ and $[\dot{\underline{\mathbf{x}}}^{(0)}, \dot{\overline{\mathbf{x}}}^{(0)}] \stackrel{\text{def}}{=} [\underline{\mathbf{x}}, \overline{\mathbf{x}}]$. Our goal is to prove that $\overline{\varphi}_{\mathcal{N}}(\dot{\underline{\mathbf{y}}}, \dot{\overline{\mathbf{y}}}, \underline{\mathbf{x}}, \overline{\mathbf{x}}; \dot{\theta})$ is a poly-size linear formula that implies

$$\forall \mathbf{x} \in [\underline{\mathbf{x}}, \overline{\mathbf{x}}]. \; \mathcal{N}(\mathbf{x}; \dot{\theta}) \in [\dot{\underline{\mathbf{y}}}, \dot{\overline{\mathbf{y}}}]$$

We first show that $\overline{\varphi}_{\mathcal{N}}(\dot{\underline{\mathbf{y}}}, \dot{\overline{\mathbf{y}}}, \underline{\mathbf{x}}, \overline{\mathbf{x}}; \dot{\theta})$ is a poly-size linear formula. For Parametric Linear Relaxation $\overline{\varphi}_{\mathcal{N}^{(\ell)}}(\dot{\underline{\mathbf{x}}}^{(\ell+1)}, \dot{\overline{\mathbf{x}}}^{(\ell+1)}, \dot{\underline{\mathbf{x}}}^{(\ell)}, \dot{\overline{\mathbf{x}}}^{(\ell)}; \dot{\theta}^{(\ell)})$ for any layer $\mathcal{N}^{(\ell)}$, by Definition 3.3, we have $\overline{\varphi}_{\mathcal{N}^{(\ell)}}(\dot{\underline{\mathbf{x}}}^{(\ell+1)}, \dot{\overline{\mathbf{x}}}^{(\ell+1)}, \dot{\underline{\mathbf{x}}}^{(\ell)}, \dot{\overline{\mathbf{x}}}^{(\ell)}; \dot{\theta}^{(\ell)})$ is a poly-size linear formula whose size is polynomial in the number of parameters, input and output dimensions of layer $\mathcal{N}^{(\ell)}$. Therefore, we have $\overline{\varphi}_{\mathcal{N}}(\dot{\underline{\mathbf{y}}}, \dot{\overline{\mathbf{y}}}, \underline{\mathbf{x}}, \overline{\mathbf{x}}; \dot{\theta})$ is a poly-size linear formula whose size is polynomial in the number of parameters, layers, and the input and output dimensions of each layer.

Then we show that $\overline{\varphi}_{\mathcal{N}}(\dot{\underline{\mathbf{y}}}, \dot{\overline{\mathbf{y}}}, \underline{\mathbf{x}}, \overline{\mathbf{x}}; \dot{\theta})$ implies $\forall \mathbf{x} \in [\underline{\mathbf{x}}, \overline{\mathbf{x}}]. \; \mathcal{N}(\mathbf{x}; \dot{\theta}) \in [\dot{\underline{\mathbf{y}}}, \dot{\overline{\mathbf{y}}}]$. By Definition 3.3, we have that the Parametric Linear Relaxation $\overline{\varphi}_{\mathcal{N}^{(\ell)}}(\dot{\underline{\mathbf{x}}}^{(\ell+1)}, \dot{\overline{\mathbf{x}}}^{(\ell+1)}, \dot{\underline{\mathbf{x}}}^{(\ell)}, \dot{\overline{\mathbf{x}}}^{(\ell)}; \dot{\theta}^{(\ell)})$ for any layer $\mathcal{N}^{(\ell)}$ implies the following formula:

$$\forall \mathbf{x}^{(\ell)} \in [\dot{\underline{\mathbf{x}}}^{(\ell)}, \dot{\overline{\mathbf{x}}}^{(\ell)}]. \; \mathcal{N}^{(\ell)}(\mathbf{x}^{(\ell)}; \dot{\theta}^{(\ell)}) \in [\dot{\underline{\mathbf{x}}}^{(\ell+1)}, \dot{\overline{\mathbf{x}}}^{(\ell+1)}]$$

Therefore, we have $\overline{\varphi}_{\mathcal{N}}(\dot{\underline{\mathbf{y}}}, \dot{\overline{\mathbf{y}}}, \underline{\mathbf{x}}, \overline{\mathbf{x}}; \dot{\theta}) \stackrel{\text{def}}{=} \bigwedge_{0 \leq \ell < L} \overline{\varphi}_{\mathcal{N}^{(\ell)}}(\dot{\underline{\mathbf{x}}}^{(\ell+1)}, \dot{\overline{\mathbf{x}}}^{(\ell+1)}, \dot{\underline{\mathbf{x}}}^{(\ell)}, \dot{\overline{\mathbf{x}}}^{(\ell)}; \dot{\theta}^{(\ell)})$ implies the following formula:

$$\bigwedge_{0 \leq \ell < L} \forall \mathbf{x}^{(\ell)} \in [\dot{\underline{\mathbf{x}}}^{(\ell)}, \dot{\overline{\mathbf{x}}}^{(\ell)}]. \; \mathcal{N}^{(\ell)}(\mathbf{x}^{(\ell)}; \dot{\theta}^{(\ell)}) \in [\dot{\underline{\mathbf{x}}}^{(\ell+1)}, \dot{\overline{\mathbf{x}}}^{(\ell+1)}]$$

where $[\dot{\underline{\mathbf{y}}}, \dot{\overline{\mathbf{y}}}] \stackrel{\text{def}}{=} [\dot{\underline{\mathbf{x}}}^{(L)}, \dot{\overline{\mathbf{x}}}^{(L)}]$ and $[\dot{\underline{\mathbf{x}}}^{(0)}, \dot{\overline{\mathbf{x}}}^{(0)}] \stackrel{\text{def}}{=} [\underline{\mathbf{x}}, \overline{\mathbf{x}}]$. By induction, the formula above implies $\forall \mathbf{x} \in [\underline{\mathbf{x}}, \overline{\mathbf{x}}]. \; \mathcal{N}(\mathbf{x}; \dot{\theta}) \in [\dot{\underline{\mathbf{y}}}, \dot{\overline{\mathbf{y}}}]$. Therefore, we proved that $\overline{\varphi}_{\mathcal{N}}(\dot{\underline{\mathbf{y}}}, \dot{\overline{\mathbf{y}}}, \underline{\mathbf{x}}, \overline{\mathbf{x}}; \dot{\theta})$ is a poly-size linear formula that implies $\forall \mathbf{x} \in [\underline{\mathbf{x}}, \overline{\mathbf{x}}]. \; \mathcal{N}(\mathbf{x}; \dot{\theta}) \in [\dot{\underline{\mathbf{y}}}, \dot{\overline{\mathbf{y}}}]$. $\qquad\square$

## A.1 Proofs for Parametric Linear Relaxation of ReLU layers

**Lemma A.1.** $\overline{\varphi}_{\text{ReLU}}(\dot{\mathbf{y}}, \ddot{\mathbf{y}}, \underline{\dot{\mathbf{x}}}, \ddot{\mathbf{x}})$ *defined in Definition 3.8 is a poly-size linear formula.*

*Proof.* As seen in Definition 3.8, $\overline{\varphi}_{\text{ReLU}}(\dot{\mathbf{y}}, \ddot{\mathbf{y}}, \underline{\dot{\mathbf{x}}}, \ddot{\mathbf{x}})$ is a linear formula. Let $m$ be the number of input dimensions of ReLU. $\overline{\varphi}_{\text{ReLU}}(\dot{\mathbf{y}}, \ddot{\mathbf{y}}, \underline{\dot{\mathbf{x}}}, \ddot{\mathbf{x}})$ has $3m$ linear constraints over $4m$ variables ($\underline{\dot{\mathbf{x}}} \in \mathbb{R}^m$, $\ddot{\mathbf{x}} \in \mathbb{R}^m$, $\dot{\mathbf{y}} \in \mathbb{R}^m$ and $\ddot{\mathbf{y}} \in \mathbb{R}^m$). Hence, $\overline{\varphi}_{\text{ReLU}}(\dot{\mathbf{y}}, \ddot{\mathbf{y}}, \underline{\dot{\mathbf{x}}}, \ddot{\mathbf{x}})$ is a poly-size linear formula whose size is polynomial in the number of input dimensions of the ReLU layer. □

**Lemma A.2.** $\overline{\varphi}_{\text{ReLU}}(\dot{\mathbf{y}}, \ddot{\mathbf{y}}, \underline{\dot{\mathbf{x}}}, \ddot{\mathbf{x}}) \overset{\text{def}}{=} \overline{\varphi}_{\text{ReLU}}(\ddot{\mathbf{y}}, \ddot{\mathbf{x}}) \wedge \underline{\varphi}_{\text{ReLU}}(\dot{\mathbf{y}}, \underline{\dot{\mathbf{x}}})$ *defined in Definition 3.8 implies* $\forall \mathbf{x} \in [\underline{\dot{\mathbf{x}}}, \ddot{\mathbf{x}}]. \text{ReLU}(\mathbf{x}) \in [\dot{\mathbf{y}}, \ddot{\mathbf{y}}]$.

*Proof.* **For the upper bound**, recall that by Definition 3.8, we have

$$\overline{\varphi}_{\text{ReLU}}(\ddot{\mathbf{y}}, \ddot{\mathbf{x}}) \overset{\text{def}}{=} \ddot{\mathbf{y}} \geq \ddot{\mathbf{x}} \wedge \ddot{\mathbf{y}} \geq 0$$

Our goal is to prove that $\overline{\varphi}_{\text{ReLU}}(\ddot{\mathbf{y}}, \ddot{\mathbf{x}})$ implies

$$\forall \mathbf{x} \in [\underline{\dot{\mathbf{x}}}, \ddot{\mathbf{x}}]. \ddot{\mathbf{y}} \geq \text{ReLU}(\mathbf{x})$$

which is equivalent to

$$\ddot{\mathbf{y}} \geq \max\left\{ \text{ReLU}(\mathbf{x}) \,\middle|\, \underline{\dot{\mathbf{x}}} \leq \mathbf{x} \leq \ddot{\mathbf{x}} \right\}$$

By the definition of the ReLU layer (Definition 3.7), we expand the formula above to

$$\ddot{\mathbf{y}} \geq \max\left\{ \max\{\mathbf{x}, 0\} \,\middle|\, \underline{\dot{\mathbf{x}}} \leq \mathbf{x} \leq \ddot{\mathbf{x}} \right\}$$

Because $\max\{\mathbf{x}, 0\}$ increases monotonically with respect to $\mathbf{x}$, the above formula is equivalent to

$$\ddot{\mathbf{y}} \geq \max\{\ddot{\mathbf{x}}, 0\}$$

Because $\max\{\ddot{\mathbf{x}}, 0\}$ is a convex function, the above formula is equivalent to

$$\ddot{\mathbf{y}} \geq \ddot{\mathbf{x}} \wedge \ddot{\mathbf{y}} \geq 0$$

which is the definition of $\overline{\varphi}_{\text{ReLU}}(\ddot{\mathbf{y}}, \ddot{\mathbf{x}})$ in Definition 3.8. Therefore, we proved that $\overline{\varphi}_{\text{ReLU}}(\ddot{\mathbf{y}}, \ddot{\mathbf{x}})$ implies $\forall \mathbf{x} \in [\underline{\dot{\mathbf{x}}}, \ddot{\mathbf{x}}]. \ddot{\mathbf{y}} \geq \text{ReLU}(\mathbf{x})$.

**For the lower bound**, recall that by Definition 3.8, we have

$$\underline{\varphi}_{\text{ReLU}}(\dot{\mathbf{y}}, \underline{\dot{\mathbf{x}}}) \overset{\text{def}}{=} \dot{\mathbf{y}} = \mathbf{c}\underline{\dot{\mathbf{x}}}$$

where $\mathbf{c} \in \mathbb{R}^m$ are constants such that $0 \leq \mathbf{c}_i \leq 1$. Our goal is to prove that $\underline{\varphi}_{\text{ReLU}}(\dot{\mathbf{y}}, \underline{\dot{\mathbf{x}}})$ implies

$$\forall \mathbf{x} \in [\underline{\dot{\mathbf{x}}}, \ddot{\mathbf{x}}]. \dot{\mathbf{y}} \leq \text{ReLU}(\mathbf{x})$$

which is equivalent to

$$\dot{\mathbf{y}} \leq \min\left\{ \text{ReLU}(\mathbf{x}) \,\middle|\, \underline{\dot{\mathbf{x}}} \leq \mathbf{x} \leq \ddot{\mathbf{x}} \right\}$$

By the definition of the ReLU layer (Definition 3.7), we expand the formula above to

$$\dot{\mathbf{y}} \leq \min\left\{ \max\{\mathbf{x}, 0\} \,\middle|\, \underline{\dot{\mathbf{x}}} \leq \mathbf{x} \leq \ddot{\mathbf{x}} \right\}$$

Because $\max\{\mathbf{x}, 0\}$ increases monotonically with respect to $\mathbf{x}$, the above formula is equivalent to

$$\dot{\mathbf{y}} \leq \max\{\underline{\dot{\mathbf{x}}}, 0\}$$

which is equivalent to

$$\dot{\mathbf{y}} \leq \underline{\dot{\mathbf{x}}} \vee \dot{\mathbf{y}} \leq 0$$

Given constant vector $\mathbf{c}$ where $0 \leq \mathbf{c} \leq 1$, $\underline{\varphi}_{\text{ReLU}}(\dot{\mathbf{y}}, \underline{\dot{\mathbf{x}}}) \overset{\text{def}}{=} \dot{\mathbf{y}} = \mathbf{c}\underline{\dot{\mathbf{x}}}$ implies the above formula. Therefore, we proved that $\underline{\varphi}_{\text{ReLU}}(\dot{\mathbf{y}}, \underline{\dot{\mathbf{x}}})$ implies $\forall \mathbf{x} \in [\underline{\dot{\mathbf{x}}}, \ddot{\mathbf{x}}]. \dot{\mathbf{y}} \leq \text{ReLU}(\mathbf{x})$. □

**Lemma A.3.** $\overline{\underline{\varphi}}_{\texttt{ReLU}}(\dot{\mathbf{y}}, \ddot{\mathbf{y}}, \dot{\mathbf{x}}, \ddot{\mathbf{x}}) \overset{\text{def}}{=} \overline{\varphi}_{\texttt{ReLU}}(\ddot{\mathbf{y}}, \ddot{\mathbf{x}}) \wedge \underline{\varphi}_{\texttt{ReLU}}(\dot{\mathbf{y}}, \dot{\mathbf{x}})$ *defined in Definition 3.8 is implied by* $\ddot{\mathbf{y}} = \max\left\{\texttt{ReLU}(\mathbf{x}) \mid \dot{\mathbf{x}} \le \mathbf{x} \le \ddot{\mathbf{x}}\right\}$, *hence captures the exact upper bound.*

*Proof.* Recall that by Definition 3.8, we have

$$\overline{\varphi}_{\texttt{ReLU}}(\ddot{\mathbf{y}}, \ddot{\mathbf{x}}) \overset{\text{def}}{=} \ddot{\mathbf{y}} \ge \ddot{\mathbf{x}} \wedge \ddot{\mathbf{y}} \ge 0$$

For any solution $(\overline{\mathbf{y}}^*, \underline{\mathbf{x}}, \overline{\mathbf{x}})$ to $\ddot{\mathbf{y}} = \max\left\{\texttt{ReLU}(\mathbf{x}) \mid \dot{\mathbf{x}} \le \mathbf{x} \le \ddot{\mathbf{x}}\right\}$, we have

$$\overline{\mathbf{y}}^* \overset{\text{def}}{=} \max\left\{\texttt{ReLU}(\mathbf{x}) \mid \mathbf{x} \le \overline{\mathbf{x}}\right\}$$

By the definition of ReLU layer (Definition 3.7), because ReLU is a monotonically increasing function, the definition above is equivalent to

$$\overline{\mathbf{y}}^* \overset{\text{def}}{=} \max\left\{\overline{\mathbf{x}}, 0\right\} \tag{9}$$

To prove that $\overline{\varphi}_{\texttt{ReLU}}(\ddot{\mathbf{y}}, \ddot{\mathbf{x}})$ captures the exact upper bound, we will show that $(\overline{\mathbf{y}}^*, \overline{\mathbf{x}})$ is a solution to $\overline{\varphi}_{\texttt{ReLU}}(\ddot{\mathbf{y}}, \ddot{\mathbf{x}})$. In other words, the following instantiated formula $\overline{\varphi}_{\texttt{ReLU}}(\overline{\mathbf{y}}^*, \overline{\mathbf{x}})$ is true:

$$\overline{\varphi}_{\texttt{ReLU}}(\overline{\mathbf{y}}^*, \overline{\mathbf{x}}) \overset{\text{def}}{=} \overline{\mathbf{y}}^* \ge \overline{\mathbf{x}} \wedge \overline{\mathbf{y}}^* \ge 0$$

By substituting $\overline{\mathbf{y}}^*$ defined in Equation 9 above, our goal becomes

$$\max\left\{\overline{\mathbf{x}}, 0\right\} \ge \overline{\mathbf{x}} \wedge \max\left\{\overline{\mathbf{x}}, 0\right\} \ge 0$$

which is true. Hence, we have proved that $\overline{\underline{\varphi}}_{\texttt{ReLU}}(\dot{\mathbf{y}}, \ddot{\mathbf{y}}, \dot{\mathbf{x}}, \ddot{\mathbf{x}})$ captures the exact upper bound. $\square$

**Theorem 3.9.** *Definition 3.8 is a Parametric Linear Relaxation for the ReLU layer that captures the exact upper bound.*

*Proof.* By Lemma A.1 and Lemma A.2, we proved that $\overline{\underline{\varphi}}_{\texttt{ReLU}}(\dot{\mathbf{y}}, \ddot{\mathbf{y}}, \dot{\mathbf{x}}, \ddot{\mathbf{x}})$ is a poly-size linear formula that implies $\forall \mathbf{x} \in [\dot{\mathbf{x}}, \ddot{\mathbf{x}}]. \texttt{ReLU}(\mathbf{x}) \in [\dot{\mathbf{y}}, \ddot{\mathbf{y}}]$, hence is a Parametric Linear Relaxation (Definition 3.3) for ReLU. By Lemma A.3, we proved that $\overline{\underline{\varphi}}_{\texttt{ReLU}}(\dot{\mathbf{y}}, \ddot{\mathbf{y}}, \dot{\mathbf{x}}, \ddot{\mathbf{x}})$ is implied by $\ddot{\mathbf{y}} = \max\left\{\texttt{ReLU}(\mathbf{x}) \mid \dot{\mathbf{x}} \le \mathbf{x} \le \ddot{\mathbf{x}}\right\}$, hence captures the exact upper bound. $\square$

## A.2 Proofs for Parametric Linear Relaxation of Affine layers with constant input bounds

**Lemma A.4.** $\overline{\underline{\varphi}}_{\texttt{Aff}}(\dot{\mathbf{y}}, \ddot{\mathbf{y}}, \underline{\mathbf{x}}, \overline{\mathbf{x}}, \dot{\mathbf{W}}, \dot{\mathbf{b}})$ *defined in Definition 3.11 is a poly-size linear formula.*

*Proof.* As seen in Definition 3.11, $\overline{\underline{\varphi}}_{\texttt{Aff}}(\dot{\mathbf{y}}, \ddot{\mathbf{y}}, \underline{\mathbf{x}}, \overline{\mathbf{x}}, \dot{\mathbf{W}}, \dot{\mathbf{b}})$ is a linear formula. Let $m$ and $n$ be the number of input and output dimensions of the Affine layer. $\overline{\underline{\varphi}}_{\texttt{Aff}}(\dot{\mathbf{y}}, \ddot{\mathbf{y}}, \underline{\mathbf{x}}, \overline{\mathbf{x}}, \dot{\mathbf{W}}, \dot{\mathbf{b}})$ has $2n + 4mn$ linear constraints over $2m + 3n + 3mn$ variables ($\underline{\mathbf{x}} \in \mathbb{R}^m$, $\overline{\mathbf{x}} \in \mathbb{R}^m$, $\dot{\mathbf{b}} \in \mathbb{R}^n$, $\dot{\mathbf{y}} \in \mathbb{R}^n$, $\ddot{\mathbf{y}} \in \mathbb{R}^n$, $\dot{\mathbf{W}} \in \mathbb{R}^{n \times m}$, $\dot{\mathbf{V}} \in \mathbb{R}^{n \times m}$ and $\ddot{\mathbf{V}} \in \mathbb{R}^{n \times m}$). Hence, $\overline{\underline{\varphi}}_{\texttt{Aff}}(\dot{\mathbf{y}}, \ddot{\mathbf{y}}, \underline{\mathbf{x}}, \overline{\mathbf{x}}, \dot{\mathbf{W}}, \dot{\mathbf{b}})$ is a poly-size linear formula whose size is polynomial in the number of input and output dimensions of the Affine layer. $\square$

**Lemma A.5.** $\overline{\underline{\varphi}}_{\texttt{Aff}}(\dot{\mathbf{y}}, \ddot{\mathbf{y}}, \underline{\mathbf{x}}, \overline{\mathbf{x}}, \dot{\mathbf{W}}, \dot{\mathbf{b}}) \overset{\text{def}}{=} \overline{\varphi}_{\texttt{Aff}}(\ddot{\mathbf{y}}, \underline{\mathbf{x}}, \overline{\mathbf{x}}, \dot{\mathbf{W}}, \dot{\mathbf{b}}) \wedge \underline{\varphi}_{\texttt{Aff}}(\dot{\mathbf{y}}, \underline{\mathbf{x}}, \overline{\mathbf{x}}, \dot{\mathbf{W}}, \dot{\mathbf{b}})$ *defined in Definition 3.11 implies* $\forall \mathbf{x} \in [\underline{\mathbf{x}}, \overline{\mathbf{x}}]. \texttt{Affine}(\mathbf{x}; \dot{\mathbf{W}}, \dot{\mathbf{b}}) \in [\dot{\mathbf{y}}, \ddot{\mathbf{y}}]$.

*Proof.* **For the upper bound**, recall that by Definition 3.11, we have

$$\overline{\varphi}_{\texttt{Aff}}(\ddot{\mathbf{y}}, \underline{\mathbf{x}}, \overline{\mathbf{x}}, \dot{\mathbf{W}}, \dot{\mathbf{b}}) \overset{\text{def}}{=} \bigwedge_j \ddot{\mathbf{y}}_j = \sum_i \ddot{\mathbf{V}}_{ji} + \dot{\mathbf{b}}_j \wedge \ddot{\mathbf{V}} \ge \dot{\mathbf{W}}\underline{\mathbf{x}} \wedge \ddot{\mathbf{V}} \ge \dot{\mathbf{W}}\overline{\mathbf{x}}$$

our goal is to prove that $\overline{\varphi}_{\texttt{Aff}}(\ddot{\mathbf{y}}, \underline{\mathbf{x}}, \overline{\mathbf{x}}, \dot{\mathbf{W}}, \dot{\mathbf{b}})$ implies

$$\forall \mathbf{x} \in [\underline{\mathbf{x}}, \overline{\mathbf{x}}]. \ddot{\mathbf{y}} \ge \texttt{Affine}(\mathbf{x}; \dot{\mathbf{W}}, \dot{\mathbf{b}})$$

We prove the above formula by proving for each $\ddot{\mathbf{y}}_j$:

$$\ddot{\mathbf{y}}_j \ge \max\left\{\texttt{Affine}(\mathbf{x}; \dot{\mathbf{W}}, \dot{\mathbf{b}})_j \mid \underline{\mathbf{x}} \le \mathbf{x} \le \overline{\mathbf{x}}\right\}$$

By the definition of the Affine layer (Definition 3.10), we expand the formula above to

$$\overline{\dot{\mathbf{y}}}_j \geq \max \left\{ \sum_{i=0}^{m} \mathbf{x}_i \dot{\mathbf{W}}_{ji} + \dot{\mathbf{b}}_j \;\middle|\; \underline{\mathbf{x}} \leq \mathbf{x} \leq \overline{\mathbf{x}} \right\}$$

which is implied by the following formula

$$\overline{\dot{\mathbf{y}}}_j \geq \sum_{i=0}^{m} \max \left\{ \mathbf{x}_i \dot{\mathbf{W}}_{ji} \;\middle|\; \underline{\mathbf{x}}_i \leq \mathbf{x}_i \leq \overline{\mathbf{x}}_i \right\} + \dot{\mathbf{b}}_j$$

Because $\overline{\varphi}_{\mathtt{Aff}}(\overline{\dot{\mathbf{y}}}, \underline{\mathbf{x}}, \overline{\mathbf{x}}, \dot{\mathbf{W}}, \dot{\mathbf{b}})$ implies $\overline{\dot{\mathbf{y}}}_j = \sum_{i=0}^{m} \overline{\dot{\mathbf{V}}}_{ji} + \dot{\mathbf{b}}_j$, by substituting the left-hand-side $\overline{\dot{\mathbf{y}}}_j$ of the formula above with $\sum_{i=0}^{m} \overline{\dot{\mathbf{V}}}_{ji} + \dot{\mathbf{b}}_j$, our goal becomes:

$$\sum_{i=0}^{m} \overline{\dot{\mathbf{V}}}_{ji} + \dot{\mathbf{b}}_j \geq \sum_{i=0}^{m} \max \left\{ \mathbf{x}_i \dot{\mathbf{W}}_{ji} \;\middle|\; \underline{\mathbf{x}}_i \leq \mathbf{x}_i \leq \overline{\mathbf{x}}_i \right\} + \dot{\mathbf{b}}_j$$

By subtracting $\dot{\mathbf{b}}_j$ from both sides, our goal becomes:

$$\sum_{i=0}^{m} \overline{\dot{\mathbf{V}}}_{ji} \geq \sum_{i=0}^{m} \max \left\{ \mathbf{x}_i \dot{\mathbf{W}}_{ji} \;\middle|\; \underline{\mathbf{x}}_i \leq \mathbf{x}_i \leq \overline{\mathbf{x}}_i \right\}$$

Then we prove the above formula by proving for each $i$:

$$\overline{\dot{\mathbf{V}}}_{ji} \geq \max \left\{ \mathbf{x}_i \dot{\mathbf{W}}_{ji} \;\middle|\; \underline{\mathbf{x}}_i \leq \mathbf{x}_i \leq \overline{\mathbf{x}}_i \right\}$$

Because $\mathbf{x}_i \dot{\mathbf{W}}_{ji}$ changes monotonically with respect to $\mathbf{x}_i$, $\max \left\{ \mathbf{x}_i \dot{\mathbf{W}}_{ji} \;\middle|\; \underline{\mathbf{x}}_i \leq \mathbf{x}_i \leq \overline{\mathbf{x}}_i \right\}$ is taken when $\mathbf{x}_i = \underline{\mathbf{x}}_i$ or $\mathbf{x}_i = \overline{\mathbf{x}}_i$. In other words, $\max \left\{ \mathbf{x}_i \dot{\mathbf{W}}_{ji} \;\middle|\; \underline{\mathbf{x}}_i \leq \mathbf{x}_i \leq \overline{\mathbf{x}}_i \right\} = \max \left\{ \underline{\mathbf{x}}_i \dot{\mathbf{W}}_{ji}, \overline{\mathbf{x}}_i \dot{\mathbf{W}}_{ji} \right\}$. By substituting the right-hand-side of the above formula, our goal becomes:

$$\overline{\dot{\mathbf{V}}}_{ji} \geq \max \left\{ \underline{\mathbf{x}}_i \dot{\mathbf{W}}_{ji}, \overline{\mathbf{x}}_i \dot{\mathbf{W}}_{ji} \right\}$$

The formula above is equivalent to $\overline{\dot{\mathbf{V}}}_{ji} \geq \underline{\mathbf{x}}_i \dot{\mathbf{W}}_{ji} \wedge \overline{\dot{\mathbf{V}}}_{ji} \geq \overline{\mathbf{x}}_i \dot{\mathbf{W}}_{ji}$, which is implied by $\overline{\varphi}_{\mathtt{Aff}}(\overline{\dot{\mathbf{y}}}, \underline{\mathbf{x}}, \overline{\mathbf{x}}, \dot{\mathbf{W}}, \dot{\mathbf{b}})$. Therefore, we have proved that $\overline{\varphi}_{\mathtt{Aff}}(\overline{\dot{\mathbf{y}}}, \underline{\mathbf{x}}, \overline{\mathbf{x}}, \dot{\mathbf{W}}, \dot{\mathbf{b}})$ implies $\forall \mathbf{x} \in [\underline{\mathbf{x}}, \overline{\mathbf{x}}].\ \overline{\dot{\mathbf{y}}} \geq \mathtt{Affine}(\mathbf{x}; \dot{\mathbf{W}}, \dot{\mathbf{b}})$.

**For the lower bound**, recall that by Definition 3.11, we have

$$\underline{\varphi}_{\mathtt{Aff}}(\underline{\dot{\mathbf{y}}}, \underline{\mathbf{x}}, \overline{\mathbf{x}}, \dot{\mathbf{W}}, \dot{\mathbf{b}}) \overset{\text{def}}{=} \bigwedge_j \underline{\dot{\mathbf{y}}}_j = \sum_i \underline{\dot{\mathbf{V}}}_{ji} + \dot{\mathbf{b}}_j \wedge \underline{\dot{\mathbf{V}}} \leq \dot{\mathbf{W}} \underline{\mathbf{x}} \wedge \underline{\dot{\mathbf{V}}} \leq \dot{\mathbf{W}} \overline{\mathbf{x}}$$

our goal is to prove that $\underline{\varphi}_{\mathtt{Aff}}(\underline{\dot{\mathbf{y}}}, \underline{\mathbf{x}}, \overline{\mathbf{x}}, \dot{\mathbf{W}}, \dot{\mathbf{b}})$ implies

$$\forall \mathbf{x} \in [\underline{\mathbf{x}}, \overline{\mathbf{x}}].\ \underline{\dot{\mathbf{y}}} \leq \mathtt{Affine}(\mathbf{x}; \dot{\mathbf{W}}, \dot{\mathbf{b}})$$

by proving for each $j$ where $\underline{\dot{\mathbf{y}}}_j$:

$$\underline{\dot{\mathbf{y}}}_j \leq \min \left\{ \mathtt{Affine}(\mathbf{x}; \dot{\mathbf{W}}, \dot{\mathbf{b}})_j \;\middle|\; \underline{\mathbf{x}} \leq \mathbf{x} \leq \overline{\mathbf{x}} \right\}$$

By the definition of the Affine layer (Definition 3.10), we expand the formula above to

$$\underline{\dot{\mathbf{y}}}_j \leq \min \left\{ \sum_{i=0}^{m} \mathbf{x}_i \dot{\mathbf{W}}_{ji} + \dot{\mathbf{b}}_j \;\middle|\; \underline{\mathbf{x}} \leq \mathbf{x} \leq \overline{\mathbf{x}} \right\}$$

which is implied by the following formula

$$\underline{\dot{\mathbf{y}}}_j \leq \sum_{i=0}^{m} \min \left\{ \mathbf{x}_i \dot{\mathbf{W}}_{ji} \;\middle|\; \underline{\mathbf{x}}_i \leq \mathbf{x}_i \leq \overline{\mathbf{x}}_i \right\} + \dot{\mathbf{b}}_j$$

Because $\underline{\varphi}_{\mathtt{Aff}}(\dot{\underline{\mathbf{y}}}, \underline{\mathbf{x}}, \overline{\mathbf{x}}, \dot{\mathbf{W}}, \dot{\mathbf{b}})$ implies $\dot{\underline{\mathbf{y}}}_j = \sum_{i=0}^{m} \dot{\underline{\mathbf{V}}}_{ji} + \dot{\mathbf{b}}_j$, by substituting the left-hand-side $\dot{\underline{\mathbf{y}}}_j$ of the formula above with $\sum_{i=0}^{m} \dot{\underline{\mathbf{V}}}_{ji} + \dot{\mathbf{b}}_j$, our goal becomes:

$$\sum_{i=0}^{m} \dot{\underline{\mathbf{V}}}_{ji} + \dot{\mathbf{b}}_j \leq \sum_{i=0}^{m} \min\left\{ \mathbf{x}_i \dot{\mathbf{W}}_{ji} \mid \underline{\mathbf{x}}_i \leq \mathbf{x}_i \leq \overline{\mathbf{x}}_i \right\} + \dot{\mathbf{b}}_j$$

By subtracting $\dot{\mathbf{b}}_j$ from both sides, our goal becomes:

$$\sum_{i=0}^{m} \dot{\underline{\mathbf{V}}}_{ji} \leq \sum_{i=0}^{m} \min\left\{ \mathbf{x}_i \dot{\mathbf{W}}_{ji} \mid \underline{\mathbf{x}}_i \leq \mathbf{x}_i \leq \overline{\mathbf{x}}_i \right\}$$

Then we will prove for each $i$:

$$\dot{\underline{\mathbf{V}}}_{ji} \leq \min\left\{ \mathbf{x}_i \dot{\mathbf{W}}_{ji} \mid \underline{\mathbf{x}}_i \leq \mathbf{x}_i \leq \overline{\mathbf{x}}_i \right\}$$

Because $\mathbf{x}_i \dot{\mathbf{W}}_{ji}$ changes monotonically with respect to $\mathbf{x}_i$, $\min\left\{ \mathbf{x}_i \dot{\mathbf{W}}_{ji} \mid \underline{\mathbf{x}}_i \leq \mathbf{x}_i \leq \overline{\mathbf{x}}_i \right\}$ is taken when $\mathbf{x}_i = \underline{\mathbf{x}}_i$ or $\mathbf{x}_i = \overline{\mathbf{x}}_i$. In other words, $\min\left\{ \mathbf{x}_i \dot{\mathbf{W}}_{ji} \mid \underline{\mathbf{x}}_i \leq \mathbf{x}_i \leq \overline{\mathbf{x}}_i \right\} = \min\left\{ \underline{\mathbf{x}}_i \dot{\mathbf{W}}_{ji}, \overline{\mathbf{x}}_i \dot{\mathbf{W}}_{ji} \right\}$. By substituting the right-hand-side of the above formula, our goal becomes:

$$\dot{\underline{\mathbf{V}}}_{ji} \leq \min\left\{ \underline{\mathbf{x}}_i \dot{\mathbf{W}}_{ji}, \overline{\mathbf{x}}_i \dot{\mathbf{W}}_{ji} \right\}$$

The formula above is equivalent to $\dot{\underline{\mathbf{V}}}_{ji} \leq \underline{\mathbf{x}}_i \dot{\mathbf{W}}_{ji} \ \wedge \ \dot{\underline{\mathbf{V}}}_{ji} \leq \overline{\mathbf{x}}_i \dot{\mathbf{W}}_{ji}$, which is implied by $\underline{\varphi}_{\mathtt{Aff}}(\dot{\underline{\mathbf{y}}}, \underline{\mathbf{x}}, \overline{\mathbf{x}}, \dot{\mathbf{W}}, \dot{\mathbf{b}})$. Therefore, we have proved that $\underline{\varphi}_{\mathtt{Aff}}(\dot{\underline{\mathbf{y}}}, \underline{\mathbf{x}}, \overline{\mathbf{x}}, \dot{\mathbf{W}}, \dot{\mathbf{b}})$ implies $\forall \mathbf{x} \in [\underline{\mathbf{x}}, \overline{\mathbf{x}}]. \ \dot{\underline{\mathbf{y}}} \leq \mathtt{Affine}(\mathbf{x}; \dot{\mathbf{W}}, \dot{\mathbf{b}})$. $\quad\square$

**Lemma A.6.** $\overline{\varphi}_{\mathtt{Aff}}(\dot{\underline{\mathbf{y}}}, \ddot{\overline{\mathbf{y}}}, \underline{\mathbf{x}}, \overline{\mathbf{x}}, \dot{\mathbf{W}}, \dot{\mathbf{b}}) \ \stackrel{\text{def}}{=} \ \overline{\varphi}_{\mathtt{Aff}}(\ddot{\overline{\mathbf{y}}}, \underline{\mathbf{x}}, \overline{\mathbf{x}}, \dot{\mathbf{W}}, \dot{\mathbf{b}}) \ \wedge \ \underline{\varphi}_{\mathtt{Aff}}(\dot{\underline{\mathbf{y}}}, \underline{\mathbf{x}}, \overline{\mathbf{x}}, \dot{\mathbf{W}}, \dot{\mathbf{b}})$ *defined in Definition 3.11 is implied by* $\ddot{\overline{\mathbf{y}}} = \max\left\{ \mathtt{Affine}(\mathbf{x}; \dot{\mathbf{W}}, \dot{\mathbf{b}}) \mid \underline{\mathbf{x}} \leq \mathbf{x} \leq \overline{\mathbf{x}} \right\} \ \wedge$ $\dot{\underline{\mathbf{y}}} = \min\left\{ \mathtt{Affine}(\mathbf{x}; \dot{\mathbf{W}}, \dot{\mathbf{b}}) \mid \underline{\mathbf{x}} \leq \mathbf{x} \leq \overline{\mathbf{x}} \right\}$, *hence captures the exact upper and lower bounds.*

*Proof.* **For the upper bound**, recall that by Definition 3.11, we have

$$\overline{\varphi}_{\mathtt{Aff}}(\ddot{\overline{\mathbf{y}}}, \underline{\mathbf{x}}, \overline{\mathbf{x}}, \dot{\mathbf{W}}, \dot{\mathbf{b}}) \ \stackrel{\text{def}}{=} \ \bigwedge_j \ddot{\overline{\mathbf{y}}}_j = \textstyle\sum_i \ddot{\overline{\mathbf{V}}}_{ji} + \dot{\mathbf{b}}_j \ \wedge \ \ddot{\overline{\mathbf{V}}} \geq \dot{\mathbf{W}}\underline{\mathbf{x}} \ \wedge \ \ddot{\overline{\mathbf{V}}} \geq \dot{\mathbf{W}}\overline{\mathbf{x}}$$

Given constant input bounds $[\underline{\mathbf{x}}, \overline{\mathbf{x}}]$, for any solution $(\overline{\mathbf{y}}^*, \mathbf{W}, \mathbf{b})$ to $\ddot{\overline{\mathbf{y}}} = \max\left\{ \mathtt{Affine}(\mathbf{x}; \dot{\mathbf{W}}, \dot{\mathbf{b}}) \mid \underline{\mathbf{x}} \leq \mathbf{x} \leq \overline{\mathbf{x}} \right\}$, we have the exact constant output upper bound defined as

$$\overline{\mathbf{y}}^* \ \stackrel{\text{def}}{=} \ \max\left\{ \mathtt{Affine}(\mathbf{x}; \mathbf{W}, \mathbf{b}) \mid \underline{\mathbf{x}} \leq \mathbf{x} \leq \overline{\mathbf{x}} \right\}$$

By the definition of Affine layer (Definition 3.10), it is equivalent to

$$\overline{\mathbf{y}}_j^* \ \stackrel{\text{def}}{=} \ \sum_i \max\left\{ \mathbf{x}_i \mathbf{W}_{ji} \mid \underline{\mathbf{x}} \leq \mathbf{x} \leq \overline{\mathbf{x}} \right\} + \mathbf{b}_j \tag{10}$$

To prove that $\overline{\varphi}_{\mathtt{Aff}}$ captures the exact upper bound, we will show that by substituting $(\overline{\mathbf{y}}^*, \underline{\mathbf{x}}, \overline{\mathbf{x}}, \mathbf{W}, \mathbf{b})$ in $\overline{\varphi}_{\mathtt{Aff}}(\dot{\underline{\mathbf{y}}}, \underline{\mathbf{x}}, \overline{\mathbf{x}}, \dot{\mathbf{W}}, \dot{\mathbf{b}})$, the instantiated formula $\overline{\varphi}_{\mathtt{Aff}}(\overline{\mathbf{y}}^*, \underline{\mathbf{x}}, \overline{\mathbf{x}}, \mathbf{W}, \mathbf{b})$, as shown below, is satisfiable:

$$\overline{\varphi}_{\mathtt{Aff}}(\overline{\mathbf{y}}^*, \underline{\mathbf{x}}, \overline{\mathbf{x}}, \mathbf{W}, \mathbf{b}) \ \stackrel{\text{def}}{=} \ \bigwedge_j \overline{\mathbf{y}}_j^* = \sum_i \ddot{\overline{\mathbf{V}}}_{ji} + \mathbf{b}_j \ \wedge \ \ddot{\overline{\mathbf{V}}} \geq \dot{\mathbf{W}}\underline{\mathbf{x}} \ \wedge \ \ddot{\overline{\mathbf{V}}} \geq \dot{\mathbf{W}}\overline{\mathbf{x}}$$

which is equivalent to prove the following formula for any index $j$:

$$\overline{\mathbf{y}}_j^* = \sum_i \ddot{\overline{\mathbf{V}}}_{ji} + \mathbf{b}_j \ \wedge \ \ddot{\overline{\mathbf{V}}} \geq \dot{\mathbf{W}}\underline{\mathbf{x}} \ \wedge \ \ddot{\overline{\mathbf{V}}} \geq \dot{\mathbf{W}}\overline{\mathbf{x}}$$

By substituting $\overline{\mathbf{y}}_j^*$ defined in Equation 10 above, our goal becomes

$$\sum_i \max\left\{ \mathbf{x}_i \mathbf{W}_{ji} \mid \underline{\mathbf{x}} \leq \mathbf{x} \leq \overline{\mathbf{x}} \right\} + \mathbf{b}_j = \sum_i \ddot{\overline{\mathbf{V}}}_{ji} + \mathbf{b}_j \ \wedge \ \ddot{\overline{\mathbf{V}}} \geq \dot{\mathbf{W}}\underline{\mathbf{x}} \ \wedge \ \ddot{\overline{\mathbf{V}}} \geq \dot{\mathbf{W}}\overline{\mathbf{x}}$$

which is implied by

$$\max\left\{\mathbf{x}_i\mathbf{W}_{ji} \mid \underline{\mathbf{x}} \le \mathbf{x} \le \overline{\mathbf{x}}\right\} = \overset{\sim}{\overline{\mathbf{V}}}_{ji} \ \wedge \ \overset{\sim}{\overline{\mathbf{V}}}_{ji} \ge \mathbf{W}_{ji}\underline{\mathbf{x}}_i \ \wedge \ \overset{\sim}{\overline{\mathbf{V}}}_{ji} \ge \mathbf{W}_{ji}\overline{\mathbf{x}}_i$$

If $\mathbf{W}_{ji} \ge 0$, $\overline{\mathbf{V}}_{ji}^* \overset{\text{def}}{=} \mathbf{W}_{ji}\overline{\mathbf{x}}_i$ is a solution to this formula; otherwise $\overline{\mathbf{V}}_{ji}^* \overset{\text{def}}{=} \mathbf{W}_{ji}\underline{\mathbf{x}}_i$ is a solution to this formula. Hence, we have proved that $\overline{\varphi}_{\texttt{Aff}}(\overline{\mathbf{y}}^*, \underline{\mathbf{x}}, \overline{\mathbf{x}}, \mathbf{W}, \mathbf{b})$ is satisfiable and $\overline{\varphi}_{\texttt{Aff}}(\dot{\overline{\mathbf{y}}}, \underline{\mathbf{x}}, \overline{\mathbf{x}}, \dot{\mathbf{W}}, \dot{\mathbf{b}})$ captures the exact upper bound.

**For the lower bound**, recall that by Definition 3.11, we have

$$\underline{\varphi}_{\texttt{Aff}}(\dot{\underline{\mathbf{y}}}, \underline{\mathbf{x}}, \overline{\mathbf{x}}, \dot{\mathbf{W}}, \dot{\mathbf{b}}) \overset{\text{def}}{=} \bigwedge_j \dot{\underline{\mathbf{y}}}_j = \sum_i \dot{\underline{\mathbf{V}}}_{ji} + \dot{\mathbf{b}}_j \ \wedge \ \dot{\underline{\mathbf{V}}} \le \dot{\mathbf{W}}\underline{\mathbf{x}} \ \wedge \ \dot{\underline{\mathbf{V}}} \le \dot{\mathbf{W}}\overline{\mathbf{x}}$$

Given constant input bounds $[\underline{\mathbf{x}}, \overline{\mathbf{x}}]$, for any solution $(\mathbf{y}^*, \mathbf{W}, \mathbf{b})$ to $\dot{\underline{\mathbf{y}}} = \min\left\{\texttt{Affine}(\mathbf{x}; \dot{\mathbf{W}}, \dot{\mathbf{b}}) \mid \underline{\mathbf{x}} \le \mathbf{x} \le \overline{\mathbf{x}}\right\}$, we have the exact constant output lower bound defined as

$$\underline{\mathbf{y}}^* \overset{\text{def}}{=} \min\left\{\texttt{Affine}(\mathbf{x}; \mathbf{W}, \mathbf{b}) \mid \underline{\mathbf{x}} \le \mathbf{x} \le \overline{\mathbf{x}}\right\}$$

By the definition of Affine layer (Definition 3.10), it is equivalent to

$$\underline{\mathbf{y}}_j^* \overset{\text{def}}{=} \sum_i \min\left\{\mathbf{x}_i\mathbf{W}_{ji} \mid \underline{\mathbf{x}} \le \mathbf{x} \le \overline{\mathbf{x}}\right\} + \mathbf{b}_j \tag{11}$$

To prove that $\underline{\varphi}_{\texttt{Aff}}$ captures the exact lower bound, we will show that by substituting $(\underline{\mathbf{y}}^*, \underline{\mathbf{x}}, \overline{\mathbf{x}}, \mathbf{W}, \mathbf{b})$ in $\underline{\varphi}_{\texttt{Aff}}(\dot{\underline{\mathbf{y}}}, \underline{\mathbf{x}}, \overline{\mathbf{x}}, \dot{\mathbf{W}}, \dot{\mathbf{b}})$, the instantiated formula $\underline{\varphi}_{\texttt{Aff}}(\underline{\mathbf{y}}^*, \underline{\mathbf{x}}, \overline{\mathbf{x}}, \mathbf{W}, \mathbf{b})$, as shown below, is satisfiable:

$$\underline{\varphi}_{\texttt{Aff}}(\underline{\mathbf{y}}^*, \underline{\mathbf{x}}, \overline{\mathbf{x}}, \mathbf{W}, \mathbf{b}) \overset{\text{def}}{=} \bigwedge_j \underline{\mathbf{y}}_j^* = \sum_i \dot{\underline{\mathbf{V}}}_{ji} + \mathbf{b}_j \ \wedge \ \dot{\underline{\mathbf{V}}} \le \mathbf{W}\underline{\mathbf{x}} \ \wedge \ \dot{\underline{\mathbf{V}}} \le \mathbf{W}\overline{\mathbf{x}}$$

which is equivalent to prove the following formula for any index $j$:

$$\underline{\mathbf{y}}_j^* = \sum_i \dot{\underline{\mathbf{V}}}_{ji} + \mathbf{b}_j \ \wedge \ \dot{\underline{\mathbf{V}}} \le \mathbf{W}\underline{\mathbf{x}} \ \wedge \ \dot{\underline{\mathbf{V}}} \le \mathbf{W}\overline{\mathbf{x}}$$

By substituting $\underline{\mathbf{y}}_j^*$ defined in Equation 11 above, our goal becomes

$$\sum_i \min\left\{\mathbf{x}_i\mathbf{W}_{ji} \mid \underline{\mathbf{x}} \le \mathbf{x} \le \overline{\mathbf{x}}\right\} + \mathbf{b}_j = \sum_i \dot{\underline{\mathbf{V}}}_{ji} + \mathbf{b}_j \ \wedge \ \dot{\underline{\mathbf{V}}} \le \mathbf{W}\underline{\mathbf{x}} \ \wedge \ \dot{\underline{\mathbf{V}}} \le \mathbf{W}\overline{\mathbf{x}}$$

which is implied by

$$\min\left\{\mathbf{x}_i\mathbf{W}_{ji} \mid \underline{\mathbf{x}} \le \mathbf{x} \le \overline{\mathbf{x}}\right\} = \dot{\underline{\mathbf{V}}}_{ji} \ \wedge \ \dot{\underline{\mathbf{V}}}_{ji} \le \mathbf{W}_{ji}\underline{\mathbf{x}}_i \ \wedge \ \dot{\underline{\mathbf{V}}}_{ji} \le \mathbf{W}_{ji}\overline{\mathbf{x}}_i$$

If $\mathbf{W}_{ji} \ge 0$, $\underline{\mathbf{V}}_{ji}^* \overset{\text{def}}{=} \mathbf{W}_{ji}\underline{\mathbf{x}}_i$ is a solution to this formula; otherwise $\underline{\mathbf{V}}_{ji}^* \overset{\text{def}}{=} \mathbf{W}_{ji}\overline{\mathbf{x}}_i$ is a solution to this formula. Hence, we have proved that $\underline{\varphi}_{\texttt{Aff}}(\underline{\mathbf{y}}^*, \underline{\mathbf{x}}, \overline{\mathbf{x}}, \mathbf{W}, \mathbf{b})$ is satisfiable and $\underline{\varphi}_{\texttt{Aff}}(\dot{\underline{\mathbf{y}}}, \underline{\mathbf{x}}, \overline{\mathbf{x}}, \dot{\mathbf{W}}, \dot{\mathbf{b}})$ captures the exact lower bound. $\square$

**Theorem 3.12.** *Definition 3.11 is a Parametric Linear Relaxation for the Affine layer with constant input bounds $[\underline{\mathbf{x}}, \overline{\mathbf{x}}]$ that captures the exact lower and upper bounds.*

*Proof.* By Lemma A.4 and Lemma A.5, we proved that $\overline{\varphi}_{\texttt{Aff}}(\dot{\underline{\mathbf{y}}}, \dot{\overline{\mathbf{y}}}, \underline{\mathbf{x}}, \overline{\mathbf{x}}, \dot{\mathbf{W}}, \dot{\mathbf{b}})$ is a poly-size linear formula that implies $\forall \mathbf{x} \in [\underline{\mathbf{x}}, \overline{\mathbf{x}}]$. $\texttt{Affine}(\mathbf{x}; \dot{\mathbf{W}}, \dot{\mathbf{b}}) \in [\dot{\underline{\mathbf{y}}}, \dot{\overline{\mathbf{y}}}]$, hence is a Parametric Linear Relaxation (Definition 3.3) for the Affine layer. By Lemma A.6, we proved that $\overline{\varphi}_{\texttt{Aff}}(\dot{\underline{\mathbf{y}}}, \dot{\overline{\mathbf{y}}}, \underline{\mathbf{x}}, \overline{\mathbf{x}}, \dot{\mathbf{W}}, \dot{\mathbf{b}})$ is implied by $\dot{\overline{\mathbf{y}}} = \max\left\{\texttt{Affine}(\mathbf{x}; \dot{\mathbf{W}}, \dot{\mathbf{b}}) \mid \underline{\mathbf{x}} \le \mathbf{x} \le \overline{\mathbf{x}}\right\} \wedge \dot{\underline{\mathbf{y}}} = \min\left\{\texttt{Affine}(\mathbf{x}; \dot{\mathbf{W}}, \dot{\mathbf{b}}) \mid \underline{\mathbf{x}} \le \mathbf{x} \le \overline{\mathbf{x}}\right\}$, hence captures the exact upper and lower bounds. $\square$

## A.3 Proofs for Parametric Linear Relaxation of Affine layers with variable input bounds

**Lemma A.7.** $\overline{\varphi}_{\texttt{Aff}}(\dot{\mathbf{y}}, \ddot{\mathbf{y}}, \underline{\dot{\mathbf{x}}}, \ddot{\mathbf{x}}, \mathbf{W}, \dot{\mathbf{b}})$ *defined in Definition 3.13 is a poly-size linear formula.*

*Proof.* As seen in Definition 3.13, $\overline{\varphi}_{\texttt{Aff}}(\dot{\mathbf{y}}, \ddot{\mathbf{y}}, \underline{\dot{\mathbf{x}}}, \ddot{\mathbf{x}}, \mathbf{W}, \dot{\mathbf{b}})$ is a linear formula. Let $m$ and $n$ be the number of input and output dimensions of the Affine layer. $\overline{\varphi}_{\texttt{Aff}}(\dot{\mathbf{y}}, \ddot{\mathbf{y}}, \underline{\dot{\mathbf{x}}}, \ddot{\mathbf{x}}, \mathbf{W}, \dot{\mathbf{b}})$ has $2n$ linear constraints over $2m + 3n$ variables ($\underline{\dot{\mathbf{x}}} \in \mathbb{R}^m$, $\ddot{\mathbf{x}} \in \mathbb{R}^m$, $\dot{\mathbf{b}} \in \mathbb{R}^n$, $\dot{\mathbf{y}} \in \mathbb{R}^n$ and $\ddot{\mathbf{y}} \in \mathbb{R}^n$). Hence, $\overline{\varphi}_{\texttt{Aff}}(\dot{\mathbf{y}}, \ddot{\mathbf{y}}, \underline{\dot{\mathbf{x}}}, \ddot{\mathbf{x}}, \mathbf{W}, \dot{\mathbf{b}})$ is a poly-size linear formula whose size is polynomial in the number of input and output dimensions of the Affine layer. $\qquad\square$

**Lemma A.8.** $\overline{\varphi}_{\texttt{Aff}}(\dot{\mathbf{y}}, \ddot{\mathbf{y}}, \underline{\dot{\mathbf{x}}}, \ddot{\mathbf{x}}, \mathbf{W}, \dot{\mathbf{b}}) \stackrel{\text{def}}{=} \overline{\varphi}_{\texttt{Aff}}(\ddot{\mathbf{y}}, \underline{\dot{\mathbf{x}}}, \ddot{\mathbf{x}}, \mathbf{W}, \dot{\mathbf{b}}) \wedge \underline{\varphi}_{\texttt{Aff}}(\dot{\mathbf{y}}, \underline{\dot{\mathbf{x}}}, \ddot{\mathbf{x}}, \mathbf{W}, \dot{\mathbf{b}})$ *defined in Definition 3.13 implies that* $\forall \mathbf{x} \in [\underline{\dot{\mathbf{x}}}, \ddot{\mathbf{x}}].\ \texttt{Affine}(\mathbf{x}; \mathbf{W}, \dot{\mathbf{b}}) \in [\dot{\mathbf{y}}, \ddot{\mathbf{y}}].$

*Proof.* **For the upper bound**, recall that by Definition 3.13, we have

$$\overline{\varphi}_{\texttt{Aff}}(\ddot{\mathbf{y}}, \underline{\dot{\mathbf{x}}}, \ddot{\mathbf{x}}, \mathbf{W}, \dot{\mathbf{b}}) \stackrel{\text{def}}{=} \bigwedge_j \ddot{\mathbf{y}}_j = \sum_i \overline{\mathbf{V}}_{ji} + \dot{\mathbf{b}}_j \text{ where } \begin{cases} \overline{\mathbf{V}}_{ji} \stackrel{\text{def}}{=} \ddot{\mathbf{x}}_i \mathbf{W}_{ji} \text{ if } \mathbf{W}_{ji} \geq 0 \\[2mm] \overline{\mathbf{V}}_{ji} \stackrel{\text{def}}{=} \underline{\dot{\mathbf{x}}}_i \mathbf{W}_{ji} \text{ otherwise} \end{cases}$$

Our goal is to prove that $\overline{\varphi}_{\texttt{Aff}}(\ddot{\mathbf{y}}, \underline{\dot{\mathbf{x}}}, \ddot{\mathbf{x}}, \mathbf{W}, \dot{\mathbf{b}})$ implies

$$\forall \mathbf{x} \in [\underline{\dot{\mathbf{x}}}, \ddot{\mathbf{x}}].\ \ddot{\mathbf{y}} \geq \texttt{Affine}(\mathbf{x}; \mathbf{W}, \dot{\mathbf{b}})$$

We prove the above formula by proving for each $\ddot{\mathbf{y}}_j$:

$$\ddot{\mathbf{y}}_j \geq \max \left\{ \texttt{Affine}(\mathbf{x}; \mathbf{W}, \dot{\mathbf{b}})_j \mid \underline{\dot{\mathbf{x}}} \leq \mathbf{x} \leq \ddot{\mathbf{x}} \right\}$$

By the definition of the Affine layer (Definition 3.10), we expand the formula above to

$$\ddot{\mathbf{y}}_j \geq \max \left\{ \sum_{i=0}^m \mathbf{x}_i \mathbf{W}_{ji} + \dot{\mathbf{b}}_j \mid \underline{\dot{\mathbf{x}}} \leq \mathbf{x} \leq \ddot{\mathbf{x}} \right\}$$

which is implied by the following formula

$$\ddot{\mathbf{y}}_j \geq \sum_{i=0}^m \max \left\{ \mathbf{x}_i \mathbf{W}_{ji} \mid \underline{\dot{\mathbf{x}}}_i \leq \mathbf{x}_i \leq \ddot{\mathbf{x}}_i \right\} + \dot{\mathbf{b}}_j$$

By Definition 3.13, $\overline{\varphi}_{\texttt{Aff}}(\ddot{\mathbf{y}}, \underline{\dot{\mathbf{x}}}, \ddot{\mathbf{x}}, \mathbf{W}, \dot{\mathbf{b}})$ implies $\ddot{\mathbf{y}}_j = \sum_{i=0}^m \overline{\mathbf{V}}_{ji} + \dot{\mathbf{b}}_j$. By substituting the left-hand-side $\ddot{\mathbf{y}}_j$ of the formula above with $\sum_{i=0}^m \overline{\mathbf{V}}_{ji} + \dot{\mathbf{b}}_j$, our goal becomes:

$$\sum_{i=0}^m \overline{\mathbf{V}}_{ji} + \dot{\mathbf{b}}_j \geq \sum_{i=0}^m \max \left\{ \mathbf{x}_i \mathbf{W}_{ji} \mid \underline{\dot{\mathbf{x}}}_i \leq \mathbf{x}_i \leq \ddot{\mathbf{x}}_i \right\} + \dot{\mathbf{b}}_j$$

By subtracting $\dot{\mathbf{b}}_j$ from both sides, our goal becomes:

$$\sum_{i=0}^m \overline{\mathbf{V}}_{ji} \geq \sum_{i=0}^m \max \left\{ \mathbf{x}_i \mathbf{W}_{ji} \mid \underline{\dot{\mathbf{x}}}_i \leq \mathbf{x}_i \leq \ddot{\mathbf{x}}_i \right\}$$

Then we prove the above formula by proving for each $i$:

$$\overline{\mathbf{V}}_{ji} \geq \max \left\{ \mathbf{x}_i \mathbf{W}_{ji} \mid \underline{\dot{\mathbf{x}}}_i \leq \mathbf{x}_i \leq \ddot{\mathbf{x}}_i \right\}$$

For the left-hand-side, we have $\overline{\mathbf{V}}_{ji} \stackrel{\text{def}}{=} \begin{cases} \ddot{\mathbf{x}}_i \mathbf{W}_{ji} & \mathbf{W}_{ji} \geq 0 \\ \underline{\dot{\mathbf{x}}}_i \mathbf{W}_{ji} & \text{otherwise} \end{cases}$. For the right-hand-side, because $\mathbf{x}_i \mathbf{W}_{ji}$ changes monotonically with respect to $\mathbf{x}_i$, and $\mathbf{W}_{ji}$ is a given constant, we

have $\max\left\{\mathbf{x}_i\mathbf{W}_{ji} \mid \underaccent{\dot}{\mathbf{x}}_i \le \mathbf{x}_i \le \ddot{\mathbf{x}}_i\right\} = \begin{cases} \ddot{\mathbf{x}}_i\mathbf{W}_{ji} & \mathbf{W}_{ji} \ge 0 \\ \underaccent{\dot}{\mathbf{x}}_i\mathbf{W}_{ji} & \text{otherwise} \end{cases}$, which is the same as the def-

inition of the left-hand-side $\overline{\mathbf{V}}_{ji}$. Thus, we have proved that $\overline{\varphi}_{\texttt{Aff}}(\ddot{\mathbf{y}}, \underaccent{\dot}{\mathbf{x}}, \ddot{\mathbf{x}}, \mathbf{W}, \dot{\mathbf{b}})$ implies $\forall \mathbf{x} \in [\underaccent{\dot}{\mathbf{x}}, \ddot{\mathbf{x}}].\ \ddot{\mathbf{y}} \ge \texttt{Affine}(\mathbf{x}; \mathbf{W}, \dot{\mathbf{b}})$.

**For the lower bound**, recall that by Definition 3.13, we have

$$\underline{\varphi}_{\texttt{Aff}}(\underaccent{\dot}{\mathbf{y}}, \underaccent{\dot}{\mathbf{x}}, \ddot{\mathbf{x}}, \mathbf{W}, \dot{\mathbf{b}}) \overset{\text{def}}{=} \bigwedge_j \underaccent{\dot}{\mathbf{y}}_j = \sum_i \underline{\mathbf{V}}_{ji} + \dot{\mathbf{b}}_j \text{ where } \begin{cases} \underline{\mathbf{V}}_{ji} \overset{\text{def}}{=} \underaccent{\dot}{\mathbf{x}}_j \mathbf{W}_{ji} \text{ if } \mathbf{W}_{ji} \ge 0 \\ \underline{\mathbf{V}}_{ji} \overset{\text{def}}{=} \ddot{\mathbf{x}}_j \mathbf{W}_{ji} \text{ otherwise} \end{cases}$$

Our goal is to prove that $\underline{\varphi}_{\texttt{Aff}}(\underaccent{\dot}{\mathbf{y}}, \underaccent{\dot}{\mathbf{x}}, \ddot{\mathbf{x}}, \mathbf{W}, \dot{\mathbf{b}})$ implies

$$\forall \mathbf{x} \in [\underaccent{\dot}{\mathbf{x}}, \ddot{\mathbf{x}}].\ \underaccent{\dot}{\mathbf{y}} \le \texttt{Affine}(\mathbf{x}; \mathbf{W}, \dot{\mathbf{b}})$$

We prove the above formula by proving for each $\underaccent{\dot}{\mathbf{y}}_j$:

$$\underaccent{\dot}{\mathbf{y}}_j \le \min\left\{\texttt{Affine}(\mathbf{x}; \mathbf{W}, \dot{\mathbf{b}})_j \mid \underaccent{\dot}{\mathbf{x}} \le \mathbf{x} \le \ddot{\mathbf{x}}\right\}$$

By the definition of the Affine layer (Definition 3.10), we expand the formula above to

$$\underaccent{\dot}{\mathbf{y}}_j \le \min\left\{\sum_{i=0}^{m} \mathbf{x}_i\mathbf{W}_{ji} + \dot{\mathbf{b}}_j \mid \underaccent{\dot}{\mathbf{x}} \le \mathbf{x} \le \ddot{\mathbf{x}}\right\}$$

which is implied by the following formula

$$\underaccent{\dot}{\mathbf{y}}_j \le \sum_{i=0}^{m} \min\left\{\mathbf{x}_i\mathbf{W}_{ji} \mid \underaccent{\dot}{\mathbf{x}}_i \le \mathbf{x}_i \le \ddot{\mathbf{x}}_i\right\} + \dot{\mathbf{b}}_j$$

By Definition 3.13, $\underline{\varphi}_{\texttt{Aff}}(\underaccent{\dot}{\mathbf{y}}, \underaccent{\dot}{\mathbf{x}}, \ddot{\mathbf{x}}, \mathbf{W}, \dot{\mathbf{b}})$ implies $\underaccent{\dot}{\mathbf{y}}_j = \sum_{i=0}^{m} \underline{\mathbf{V}}_{ji} + \dot{\mathbf{b}}_j$. By substituting the left-hand-side $\underaccent{\dot}{\mathbf{y}}_j$ of the formula above with $\sum_{i=0}^{m} \underline{\mathbf{V}}_{ji} + \dot{\mathbf{b}}_j$, our goal becomes:

$$\sum_{i=0}^{m} \underline{\mathbf{V}}_{ji} + \dot{\mathbf{b}}_j \le \sum_{i=0}^{m} \min\left\{\mathbf{x}_i\mathbf{W}_{ji} \mid \underaccent{\dot}{\mathbf{x}}_i \le \mathbf{x}_i \le \ddot{\mathbf{x}}_i\right\} + \dot{\mathbf{b}}_j$$

By subtracting $\dot{\mathbf{b}}_j$ from both sides, our goal becomes:

$$\sum_{i=0}^{m} \underline{\mathbf{V}}_{ji} \le \sum_{i=0}^{m} \min\left\{\mathbf{x}_i\mathbf{W}_{ji} \mid \underaccent{\dot}{\mathbf{x}}_i \le \mathbf{x}_i \le \ddot{\mathbf{x}}_i\right\}$$

Then we prove the above formula by proving for each $i$:

$$\underline{\mathbf{V}}_{ji} \le \min\left\{\mathbf{x}_i\mathbf{W}_{ji} \mid \underaccent{\dot}{\mathbf{x}}_i \le \mathbf{x}_i \le \ddot{\mathbf{x}}_i\right\}$$

For the left-hand-side, we have $\underline{\mathbf{V}}_{ji} \overset{\text{def}}{=} \begin{cases} \underaccent{\dot}{\mathbf{x}}_i\mathbf{W}_{ji} & \mathbf{W}_{ji} \ge 0 \\ \ddot{\mathbf{x}}_i\mathbf{W}_{ji} & \text{otherwise} \end{cases}$. For the right-hand-side, be-

cause $\mathbf{x}_i\mathbf{W}_{ji}$ changes monotonically with respect to $\mathbf{x}_i$, and $\mathbf{W}_{ji}$ is a given constant, we

have $\min\left\{\mathbf{x}_i\mathbf{W}_{ji} \mid \underaccent{\dot}{\mathbf{x}}_i \le \mathbf{x}_i \le \ddot{\mathbf{x}}_i\right\} = \begin{cases} \underaccent{\dot}{\mathbf{x}}_i\mathbf{W}_{ji} & \mathbf{W}_{ji} \ge 0 \\ \ddot{\mathbf{x}}_i\mathbf{W}_{ji} & \text{otherwise} \end{cases}$, which is the same as the def-

inition of the left-hand-side $\underline{\mathbf{V}}_{ji}$. Thus, we have proved that $\underline{\varphi}_{\texttt{Aff}}(\underaccent{\dot}{\mathbf{y}}, \underaccent{\dot}{\mathbf{x}}, \ddot{\mathbf{x}}, \mathbf{W}, \dot{\mathbf{b}})$ implies $\forall \mathbf{x} \in [\underaccent{\dot}{\mathbf{x}}, \ddot{\mathbf{x}}].\ \underaccent{\dot}{\mathbf{y}} \le \texttt{Affine}(\mathbf{x}; \mathbf{W}, \dot{\mathbf{b}})$. $\qquad\square$

**Lemma A.9.** $\overline{\underline{\varphi}}_{\texttt{Aff}}(\underaccent{\dot}{\mathbf{y}}, \ddot{\mathbf{y}}, \underaccent{\dot}{\mathbf{x}}, \ddot{\mathbf{x}}, \mathbf{W}, \dot{\mathbf{b}}) \overset{\text{def}}{=} \overline{\varphi}_{\texttt{Aff}}(\ddot{\mathbf{y}}, \underaccent{\dot}{\mathbf{x}}, \ddot{\mathbf{x}}, \mathbf{W}, \dot{\mathbf{b}}) \wedge \underline{\varphi}_{\texttt{Aff}}(\underaccent{\dot}{\mathbf{y}}, \underaccent{\dot}{\mathbf{x}}, \ddot{\mathbf{x}}, \mathbf{W}, \dot{\mathbf{b}})$ de-
fined in Definition 3.13 is implied by $\ddot{\mathbf{y}} = \max\left\{\texttt{Affine}(\mathbf{x}; \mathbf{W}, \dot{\mathbf{b}}) \mid \underaccent{\dot}{\mathbf{x}} \le \mathbf{x} \le \ddot{\mathbf{x}}\right\} \wedge$
$\underaccent{\dot}{\mathbf{y}} = \min\left\{\texttt{Affine}(\mathbf{x}; \mathbf{W}, \dot{\mathbf{b}}) \mid \underaccent{\dot}{\mathbf{x}} \le \mathbf{x} \le \ddot{\mathbf{x}}\right\}$, hence captures the exact upper and lower bounds.

*Proof.* **For the upper bound**, recall that by Definition 3.13, we have

$$\overline{\varphi}_{\mathtt{Aff}}(\ddot{\mathbf{y}}, \underline{\dot{\mathbf{x}}}, \ddot{\mathbf{x}}, \mathbf{W}, \dot{\mathbf{b}}) \stackrel{\text{def}}{=} \bigwedge_j \ddot{\mathbf{y}}_j = \sum_i \overline{\mathbf{V}}_{ji} + \dot{\mathbf{b}}_j \text{ where } \begin{cases} \overline{\mathbf{V}}_{ji} \stackrel{\text{def}}{=} \ddot{\mathbf{x}}_i \mathbf{W}_{ji} & \text{if } \mathbf{W}_{ji} \geq 0 \\ \overline{\mathbf{V}}_{ji} \stackrel{\text{def}}{=} \underline{\dot{\mathbf{x}}}_i \mathbf{W}_{ji} & \text{otherwise} \end{cases}$$

For any solution $(\overline{\mathbf{y}}^*, \underline{\mathbf{x}}, \overline{\mathbf{x}}, \mathbf{b})$ to $\ddot{\mathbf{y}} = \max\left\{ \mathtt{Affine}(\mathbf{x}; \mathbf{W}, \dot{\mathbf{b}}) \mid \underline{\dot{\mathbf{x}}} \leq \mathbf{x} \leq \ddot{\mathbf{x}} \right\}$, we have the exact constant output upper bound defined as

$$\overline{\mathbf{y}}^* \stackrel{\text{def}}{=} \max\left\{ \mathtt{Affine}(\mathbf{x}; \mathbf{W}, \mathbf{b}) \mid \underline{\mathbf{x}} \leq \mathbf{x} \leq \overline{\mathbf{x}} \right\}$$

By the definition of Affine layer (Definition 3.10), it is equivalent to

$$\overline{\mathbf{y}}_j^* \stackrel{\text{def}}{=} \sum_i \max\left\{ \mathbf{x}_i \mathbf{W}_{ji} \mid \underline{\mathbf{x}} \leq \mathbf{x} \leq \overline{\mathbf{x}} \right\} + \mathbf{b}_j \tag{12}$$

To prove that $\overline{\varphi}_{\mathtt{Aff}}$ captures the exact upper bound, we will show that $(\overline{\mathbf{y}}^*, \underline{\mathbf{x}}, \overline{\mathbf{x}}, \mathbf{b})$ is a solution to $\overline{\varphi}_{\mathtt{Aff}}(\ddot{\mathbf{y}}, \underline{\dot{\mathbf{x}}}, \ddot{\mathbf{x}}, \mathbf{W}, \dot{\mathbf{b}})$. In other words, the following instantiated formula $\overline{\varphi}_{\mathtt{Aff}}(\overline{\mathbf{y}}^*, \underline{\mathbf{x}}, \overline{\mathbf{x}}, \mathbf{W}, \mathbf{b})$ is true:

$$\overline{\varphi}_{\mathtt{Aff}}(\overline{\mathbf{y}}^*, \underline{\mathbf{x}}, \overline{\mathbf{x}}, \mathbf{W}, \mathbf{b}) \stackrel{\text{def}}{=} \bigwedge_j \overline{\mathbf{y}}_j^* = \sum_i \overline{\mathbf{V}}_{ji} + \mathbf{b}_j$$

where $\overline{\mathbf{V}}_{ji} \stackrel{\text{def}}{=} \overline{\mathbf{x}}_j \mathbf{W}_{ji}$ if $\mathbf{W}_{ji} \geq 0$, otherwise $\overline{\mathbf{V}}_{ji} \stackrel{\text{def}}{=} \underline{\mathbf{x}}_j \mathbf{W}_{ji}$.

This formula is equivalent to prove the following formula for any index $j$:

$$\overline{\mathbf{y}}_j^* = \sum_i \overline{\mathbf{V}}_{ji} + \mathbf{b}_j$$

By substituting $\overline{\mathbf{y}}_j^*$ defined in Equation 12 above, our goal becomes

$$\sum_i \max\left\{ \mathbf{x}_i \mathbf{W}_{ji} \mid \underline{\mathbf{x}} \leq \mathbf{x} \leq \overline{\mathbf{x}} \right\} + \mathbf{b}_j = \sum_i \overline{\mathbf{V}}_{ji} + \mathbf{b}_j$$

which is implied by

$$\max\left\{ \mathbf{x}_i \mathbf{W}_{ji} \mid \underline{\mathbf{x}} \leq \mathbf{x} \leq \overline{\mathbf{x}} \right\} = \overline{\mathbf{V}}_{ji}$$

If $\mathbf{W}_{ji} \geq 0$, $\max\left\{ \mathbf{x}_i \mathbf{W}_{ji} \mid \underline{\mathbf{x}} \leq \mathbf{x} \leq \overline{\mathbf{x}} \right\} = \overline{\mathbf{x}}_j \mathbf{W}_{ji} = \overline{\mathbf{V}}_{ji}$, the above formula is true; Otherwise, $\max\left\{ \mathbf{x}_i \mathbf{W}_{ji} \mid \underline{\mathbf{x}} \leq \mathbf{x} \leq \overline{\mathbf{x}} \right\} = \underline{\mathbf{x}}_j \mathbf{W}_{ji} = \overline{\mathbf{V}}_{ji}$, the above formula is true. Hence, we have proved that $\overline{\varphi}_{\mathtt{Aff}}(\overline{\mathbf{y}}^*, \underline{\mathbf{x}}, \overline{\mathbf{x}}, \mathbf{W}, \mathbf{b})$ is true and $\overline{\varphi}_{\mathtt{Aff}}(\ddot{\mathbf{y}}, \underline{\dot{\mathbf{x}}}, \ddot{\mathbf{x}}, \mathbf{W}, \dot{\mathbf{b}})$ captures the exact upper bound.

**For the lower bound**, recall that by Definition 3.13, we have

$$\underline{\varphi}_{\mathtt{Aff}}(\dot{\mathbf{y}}, \underline{\dot{\mathbf{x}}}, \ddot{\mathbf{x}}, \mathbf{W}, \dot{\mathbf{b}}) \stackrel{\text{def}}{=} \bigwedge_j \dot{\mathbf{y}}_j = \sum_i \underline{\mathbf{V}}_{ji} + \dot{\mathbf{b}}_j \text{ where } \begin{cases} \underline{\mathbf{V}}_{ji} \stackrel{\text{def}}{=} \underline{\dot{\mathbf{x}}}_j \mathbf{W}_{ji} & \text{if } \mathbf{W}_{ji} \geq 0 \\ \underline{\mathbf{V}}_{ji} \stackrel{\text{def}}{=} \ddot{\mathbf{x}}_j \mathbf{W}_{ji} & \text{otherwise} \end{cases}$$

For any solution $(\mathbf{y}^*, \underline{\mathbf{x}}, \overline{\mathbf{x}}, \mathbf{b})$ to $\dot{\mathbf{y}} = \min\left\{ \mathtt{Affine}(\mathbf{x}; \mathbf{W}, \dot{\mathbf{b}}) \mid \underline{\dot{\mathbf{x}}} \leq \mathbf{x} \leq \ddot{\mathbf{x}} \right\}$, we have the exact constant output lower bound defined as

$$\underline{\mathbf{y}}^* \stackrel{\text{def}}{=} \min\left\{ \mathtt{Affine}(\mathbf{x}; \mathbf{W}, \mathbf{b}) \mid \underline{\mathbf{x}} \leq \mathbf{x} \leq \overline{\mathbf{x}} \right\}$$

By the definition of Affine layer (Definition 3.10), it is equivalent to

$$\underline{\mathbf{y}}_j^* \stackrel{\text{def}}{=} \sum_i \min\left\{ \mathbf{x}_i \mathbf{W}_{ji} \mid \underline{\mathbf{x}} \leq \mathbf{x} \leq \overline{\mathbf{x}} \right\} + \mathbf{b}_j \tag{13}$$

To prove that $\underline{\varphi}_{\mathtt{Aff}}$ captures the exact lower bound, we will show that $(\underline{\mathbf{y}}^*, \underline{\mathbf{x}}, \overline{\mathbf{x}}, \mathbf{b})$ is a solution to $\underline{\varphi}_{\mathtt{Aff}}(\dot{\mathbf{y}}, \underline{\dot{\mathbf{x}}}, \ddot{\mathbf{x}}, \mathbf{W}, \dot{\mathbf{b}})$. In other words, the following instantiated formula $\underline{\varphi}_{\mathtt{Aff}}(\underline{\mathbf{y}}^*, \underline{\mathbf{x}}, \overline{\mathbf{x}}, \mathbf{W}, \mathbf{b})$ is true:

$$\underline{\varphi}_{\mathtt{Aff}}(\underline{\mathbf{y}}^*, \underline{\mathbf{x}}, \overline{\mathbf{x}}, \mathbf{W}, \mathbf{b}) \stackrel{\text{def}}{=} \bigwedge_j \underline{\mathbf{y}}_j^* = \sum_i \underline{\mathbf{V}}_{ji} + \mathbf{b}_j$$

where $\underline{\mathbf{V}}_{ji} \overset{\text{def}}{=} \underline{\mathbf{x}}_j \mathbf{W}_{ji}$ if $\mathbf{W}_{ji} \geq 0$, otherwise $\underline{\mathbf{V}}_{ji} \overset{\text{def}}{=} \overline{\mathbf{x}}_j \mathbf{W}_{ji}$.

This formula is equivalent to prove the following formula for any index $j$:

$$\underline{\mathbf{y}}_j^* = \sum_i \underline{\mathbf{V}}_{ji} + \mathbf{b}_j$$

By substituting $\underline{\mathbf{y}}_j^*$ defined in Equation 13 above, our goal becomes

$$\sum_i \min\left\{ \mathbf{x}_i \mathbf{W}_{ji} \mid \underline{\mathbf{x}} \leq \mathbf{x} \leq \overline{\mathbf{x}} \right\} + \mathbf{b}_j = \sum_i \underline{\mathbf{V}}_{ji} + \mathbf{b}_j$$

which is implied by

$$\min\left\{ \mathbf{x}_i \mathbf{W}_{ji} \mid \underline{\mathbf{x}} \leq \mathbf{x} \leq \overline{\mathbf{x}} \right\} = \underline{\mathbf{V}}_{ji}$$

If $\mathbf{W}_{ji} \geq 0$, $\min\left\{ \mathbf{x}_i \mathbf{W}_{ji} \mid \underline{\mathbf{x}} \leq \mathbf{x} \leq \overline{\mathbf{x}} \right\} = \underline{\mathbf{x}}_j \mathbf{W}_{ji} = \underline{\mathbf{V}}_{ji}$, the above formula is true; Otherwise, $\min\left\{ \mathbf{x}_i \mathbf{W}_{ji} \mid \underline{\mathbf{x}} \leq \mathbf{x} \leq \overline{\mathbf{x}} \right\} = \overline{\mathbf{x}}_j \mathbf{W}_{ji} = \underline{\mathbf{V}}_{ji}$, the above formula is true. Hence, we have proved that $\varphi_{\texttt{Aff}}(\underline{\mathbf{y}}^*, \underline{\mathbf{x}}, \overline{\mathbf{x}}, \mathbf{W}, \dot{\mathbf{b}})$ is true and $\varphi_{\texttt{Aff}}(\dot{\underline{\mathbf{y}}}, \dot{\underline{\mathbf{x}}}, \ddot{\mathbf{x}}, \mathbf{W}, \dot{\mathbf{b}})$ captures the exact lower bound. $\qquad\square$

**Theorem 3.14.** *Definition 3.13 is a Parametric Linear Relaxation for the Affine layer with variable input bounds $[\dot{\underline{\mathbf{x}}}, \ddot{\mathbf{x}}]$ that captures the exact upper and lower bounds.*

*Proof.* By Lemma A.7 and Lemma A.8, we proved that $\overline{\varphi}_{\texttt{Aff}}(\dot{\underline{\mathbf{y}}}, \ddot{\mathbf{y}}, \dot{\underline{\mathbf{x}}}, \ddot{\mathbf{x}}, \mathbf{W}, \dot{\mathbf{b}})$ is a poly-size linear formula that implies $\forall \mathbf{x} \in [\dot{\underline{\mathbf{x}}}, \ddot{\mathbf{x}}].\ \texttt{Affine}(\mathbf{x}; \mathbf{W}, \dot{\mathbf{b}}) \in [\dot{\underline{\mathbf{y}}}, \ddot{\mathbf{y}}]$, hence is a Parametric Linear Relaxation (Definition 3.3) for the Affine layer. By Lemma A.9, we proved that $\overline{\varphi}_{\texttt{Aff}}(\dot{\underline{\mathbf{y}}}, \ddot{\mathbf{y}}, \dot{\underline{\mathbf{x}}}, \ddot{\mathbf{x}}, \mathbf{W}, \dot{\mathbf{b}})$ is implied by $\ddot{\mathbf{y}} = \max\left\{ \texttt{Affine}(\mathbf{x}; \mathbf{W}, \dot{\mathbf{b}}) \mid \dot{\underline{\mathbf{x}}} \leq \mathbf{x} \leq \ddot{\mathbf{x}} \right\} \wedge \dot{\underline{\mathbf{y}}} = \min\left\{ \texttt{Affine}(\mathbf{x}; \mathbf{W}, \dot{\mathbf{b}}) \mid \dot{\underline{\mathbf{x}}} \leq \mathbf{x} \leq \ddot{\mathbf{x}} \right\}$, hence captures the exact upper and lower bounds. $\qquad\square$

# B  General provable editing via Parametric Linear Relaxation

In this section we present our approach for solving the general provable editing problem (Definition 1.1):

**Definition B.1.** Given Parametric Linear Relaxation $\overline{\varphi}_{\mathcal{N}}(\dot{\underline{\mathbf{y}}}, \ddot{\mathbf{y}}, \underline{\mathbf{x}}, \overline{\mathbf{x}}; \dot{\theta})$ for DNN $\mathcal{N}$ with parameters $\theta$, an input polytope $\mathrm{P} \overset{\text{def}}{=} \left\{ \mathbf{x} \mid \mathbf{Dx} \leq \mathbf{e} \right\}$, an output polytope $\mathrm{Q} \overset{\text{def}}{=} \left\{ \mathbf{y} \mid \mathbf{Ay} \leq \mathbf{b} \right\}$. Given a hyperparameter $k$ so that we only edit the DNN layers $\mathcal{N}^{(k:L)}$ starting from the $k$th layer. Then a solution to the following linear program is a solution to the provable editing problem of Definition 1.1:

$$\min \|\dot{\theta} - \theta\| \quad \text{s.t.} \quad \overline{\varphi}_{\mathcal{N}^{(k:L)}}\left(\dot{\underline{\mathbf{y}}}, \ddot{\mathbf{y}}, \underline{\mathbf{x}}^{(k)}, \overline{\mathbf{x}}^{(k)}; \dot{\theta}^{(k:L)}\right) \wedge \psi(\dot{\underline{\mathbf{y}}}, \ddot{\mathbf{y}}, \mathbf{A}, \mathbf{b}) \tag{14}$$

where a **sound constant interval bound** $\left[\underline{\mathbf{x}}^{(k)}, \overline{\mathbf{x}}^{(k)}\right]$ for the input $\mathbf{x}^{(k)}$ to the slice $\mathcal{N}^{(k:L)}$ to edit is computed by a sound bound propagation tool of DNNs like `DeepPoly` [39], `auto_LiRPA` [50] and `DeepT` [4]:

$$\left[\underline{\mathbf{x}}^{(k)}, \overline{\mathbf{x}}^{(k)}\right] \overset{\text{def}}{=} \texttt{Compute\_Bound}_{\mathcal{N}^{(0:k)}}\left(\mathrm{P}; \theta^{(0:k)}\right) \tag{15}$$

The **output constraint** $\psi(\dot{\underline{\mathbf{y}}}, \ddot{\mathbf{y}}, \mathbf{A}, \mathbf{b})$ encodes $\mathbf{Ay} \leq \mathbf{b}$ for all $\mathbf{y} \in [\dot{\underline{\mathbf{y}}}, \ddot{\mathbf{y}}]$ using the parametric output bounds $[\dot{\underline{\mathbf{y}}}, \ddot{\mathbf{y}}]$ is defined as follows:

$$\psi(\dot{\underline{\mathbf{y}}}, \ddot{\mathbf{y}}, \mathbf{A}, \mathbf{b}) \overset{\text{def}}{=} \bigwedge_{i=0}^{m} \sum_{j=0}^{n} \mathbf{E}_{ij} \leq \mathbf{b}_i \quad \text{where} \quad \mathbf{E}_{ij} \overset{\text{def}}{=} \begin{cases} \mathbf{A}_{ij} \ddot{\mathbf{y}}_j & \mathbf{A}_{ij} \geq 0 \\ \mathbf{A}_{ij} \dot{\underline{\mathbf{y}}}_j & \text{otherwise} \end{cases} \tag{16} \blacksquare$$

## B.1 Provable editing for multiple properties via Parametric Linear Relaxation

Given $n$ properties defined by input polytopes $P_i \stackrel{\text{def}}{=} \{\mathbf{x} \mid {}_i\mathbf{D}\mathbf{x} \leq {}_i\mathbf{e}\}$, and output polytopes $Q_i \stackrel{\text{def}}{=} \{\mathbf{y} \mid {}_i\mathbf{A}\mathbf{y} \leq {}_i\mathbf{b}\}$. This work can directly generalize to provably edit multiple properties:

$$\min \|\dot{\boldsymbol{\theta}} - \theta\| \quad \text{s.t.} \quad \bigwedge_{0 \leq i < n} \forall \mathbf{x} \in P_i. \, \mathcal{N}(\mathbf{x}; \dot{\boldsymbol{\theta}}) \in Q_i \tag{17}$$

In particular, most of our experiments in Section 4 involve editing DNNs for multiple properties altogether. Here we present our formulation for provably editing multiple properties, which encodes the conjunction of the constraints for each property.

$$\min \|\dot{\boldsymbol{\theta}} - \theta\| \quad \text{s.t.} \quad \bigwedge_{0 \leq i < n} \overline{\underline{\varphi}}_{\mathcal{N}^{(k:L)}}\left({}_i\underline{\dot{\mathbf{y}}}, {}_i\ddot{\mathbf{y}}, {}_i\underline{\mathbf{x}}^{(k)}, {}_i\overline{\mathbf{x}}^{(k)}; \dot{\boldsymbol{\theta}}^{(k:L)}\right) \wedge \psi({}_i\underline{\dot{\mathbf{y}}}, {}_i\ddot{\mathbf{y}}, {}_i\mathbf{A}, {}_i\mathbf{b}) \tag{18}$$

where $\left[{}_i\underline{\mathbf{x}}^{(k)}, {}_i\overline{\mathbf{x}}^{(k)}\right] \stackrel{\text{def}}{=} \texttt{Compute\_Bound}_{\mathcal{N}^{(0:k)}}\left(P_i; \theta^{(0:k)}\right)$.

## B.2 Provable editing for disjunctive properties via Parametric Linear Relaxation

Given a property defined by an input polytope $P \stackrel{\text{def}}{=} \{\mathbf{x} \mid \mathbf{D}\mathbf{x} \leq \mathbf{e}\}$, and $n$ possible output polytopes $Q_j \stackrel{\text{def}}{=} \{\mathbf{y} \mid {}_j\mathbf{A}\mathbf{y} \leq {}_j\mathbf{b}\}$, this work can directly generalize to provably edit disjunctive property:

$$\min \|\dot{\boldsymbol{\theta}} - \theta\| \quad \text{s.t.} \quad \forall \mathbf{x} \in P_j. \, \bigvee_{0 \leq j < n} \mathcal{N}(\mathbf{x}; \dot{\boldsymbol{\theta}}) \in Q_j \tag{19}$$

by using a mixed-integer linear programming (MILP) solver. In particular, some properties in VNN-COMP (Section 4.1) involve editing DNNs for such disjunctive properties. Here we present our formulation:

$$\min \|\dot{\boldsymbol{\theta}} - \theta\| \quad \text{s.t.} \quad \overline{\underline{\varphi}}_{\mathcal{N}^{(k:L)}}\left(\underline{\dot{\mathbf{y}}}, \ddot{\mathbf{y}}, \underline{\mathbf{x}}^{(k)}, \overline{\mathbf{x}}^{(k)}; \dot{\boldsymbol{\theta}}^{(k:L)}\right) \wedge \bigvee_{0 \leq j < n} \psi(\underline{\dot{\mathbf{y}}}, \ddot{\mathbf{y}}, {}_j\mathbf{A}, {}_j\mathbf{b}) \tag{20}$$

where $\left[\underline{\mathbf{x}}^{(k)}, \overline{\mathbf{x}}^{(k)}\right] \stackrel{\text{def}}{=} \texttt{Compute\_Bound}_{\mathcal{N}^{(0:k)}}\left(P; \theta^{(0:k)}\right)$.

# C Parametric Linear Relaxation for Tanh, Sigmoid and ELU layers

## C.1 Parametric Linear Relaxation for Tanh layers

**Definition C.1.** $\mathbf{y} \stackrel{\text{def}}{=} \texttt{Tanh}(\mathbf{x})$ with input $\mathbf{x} \in \mathbb{R}^m$ and output $\mathbf{y} \in \mathbb{R}^m$ is defined as $\mathbf{y}_i \stackrel{\text{def}}{=} \tanh(\mathbf{x}_i)$.

**Definition C.2.** For Tanh layer with variable input bounds $[\underline{\dot{\mathbf{x}}}, \ddot{\mathbf{x}}]$, its Parametric Linear Relaxation is defined as $\overline{\underline{\varphi}}_{\texttt{Tanh}}(\underline{\dot{\mathbf{y}}}, \ddot{\mathbf{y}}, \underline{\dot{\mathbf{x}}}, \ddot{\mathbf{x}}) \stackrel{\text{def}}{=} \overline{\varphi}_{\texttt{Tanh}}(\ddot{\mathbf{y}}, \ddot{\mathbf{x}}) \wedge \underline{\varphi}_{\texttt{Tanh}}(\underline{\dot{\mathbf{y}}}, \underline{\dot{\mathbf{x}}})$ where:

$$\begin{aligned}
\overline{\varphi}_{\texttt{Tanh}}(\ddot{\mathbf{y}}, \ddot{\mathbf{x}}) &\stackrel{\text{def}}{=} \ddot{\mathbf{y}} \geq \texttt{Tanh}(\mathbf{x}^u) \wedge \ddot{\mathbf{y}} \geq \mathbf{a}^u \ddot{\mathbf{x}} + \mathbf{b}^u \\
\underline{\varphi}_{\texttt{Tanh}}(\underline{\dot{\mathbf{y}}}, \underline{\dot{\mathbf{x}}}) &\stackrel{\text{def}}{=} \underline{\dot{\mathbf{y}}} \leq \texttt{Tanh}(\mathbf{x}^l) \wedge \underline{\dot{\mathbf{y}}} \leq \mathbf{a}^l \underline{\dot{\mathbf{x}}} + \mathbf{b}^l
\end{aligned} \tag{21}$$

where $\mathbf{x}^u \in \mathbb{R}^m$, $\mathbf{a}^u \in \mathbb{R}^m$, $\mathbf{b}^u \in \mathbb{R}^m$ are constants that satisfy the following two conditions: **i)** let $f(\boldsymbol{x}) = \mathbf{a}_i^u \boldsymbol{x} + \mathbf{b}_i^u$, the line defined by function $f(\boldsymbol{x})$ is tangent to the concave piece of $\tanh$ at some $\boldsymbol{x} \geq 0$; **ii)** $\mathbf{a}^u \mathbf{x}^u + \mathbf{b}^u = \texttt{Tanh}(\mathbf{x}^u)$; viz., the line defined by function $f(\boldsymbol{x})$ intersects with convex piece of $\tanh$ at $\mathbf{x}_i^u$ where $\mathbf{x}_i^u \leq 0$.

Similarly, $\mathbf{x}^l \in \mathbb{R}^m$, $\mathbf{a}^l \in \mathbb{R}^m$, $\mathbf{b}^l \in \mathbb{R}^m$ are constants that satisfy the following two conditions: **i)** let $f(\boldsymbol{x}) = \mathbf{a}_i^l \boldsymbol{x} + \mathbf{b}_i^l$, the line defined by function $f(\boldsymbol{x})$ is tangent to the convex piece of $\tanh$ at some $\boldsymbol{x} \leq 0$; **ii)** $\mathbf{a}^l \mathbf{x}^l + \mathbf{b}^l = \texttt{Tanh}(\mathbf{x}^l)$; viz., the line defined by function $f(\boldsymbol{x})$ intersects with concave piece of $\tanh$ at $\mathbf{x}_i^l$ where $\mathbf{x}_i^l \geq 0$. ■

**Theorem C.3.** *Definition C.2 is a Parametric Linear Relaxation for the Tanh layer.*

*Proof.* As seen in Definition C.2, $\overline{\underline{\varphi}}_{\texttt{Tanh}}(\dot{\underline{\mathbf{y}}}, \ddot{\overline{\mathbf{y}}}, \dot{\underline{\mathbf{x}}}, \ddot{\overline{\mathbf{x}}})$ is a linear formula. Let $m$ be the number of input dimensions of Tanh. $\overline{\underline{\varphi}}_{\texttt{Tanh}}(\dot{\underline{\mathbf{y}}}, \ddot{\overline{\mathbf{y}}}, \dot{\underline{\mathbf{x}}}, \ddot{\overline{\mathbf{x}}})$ has $4m$ linear constraints over $4m$ variables ($\dot{\underline{\mathbf{x}}} \in \mathbb{R}^m$, $\ddot{\overline{\mathbf{x}}} \in \mathbb{R}^m$, $\dot{\underline{\mathbf{y}}} \in \mathbb{R}^m$ and $\ddot{\overline{\mathbf{y}}} \in \mathbb{R}^m$). Hence, $\overline{\underline{\varphi}}_{\texttt{Tanh}}(\dot{\underline{\mathbf{y}}}, \ddot{\overline{\mathbf{y}}}, \dot{\underline{\mathbf{x}}}, \ddot{\overline{\mathbf{x}}})$ is a poly-size linear formula, whose size is polynomial in the number of input dimensions of the Tanh layer.

Now our goal is to prove that $\overline{\underline{\varphi}}_{\texttt{Tanh}}(\dot{\underline{\mathbf{y}}}, \ddot{\overline{\mathbf{y}}}, \dot{\underline{\mathbf{x}}}, \ddot{\overline{\mathbf{x}}})$ defined in Definition C.2 implies

$$\forall \mathbf{x} \in [\dot{\underline{\mathbf{x}}}, \ddot{\overline{\mathbf{x}}}].\ \dot{\underline{\mathbf{y}}} \leq \texttt{Tanh}(\mathbf{x}) \leq \ddot{\overline{\mathbf{y}}}$$

We will discuss the upper and lower bounds separately.

**For the upper bound constraint** $\overline{\varphi}_{\texttt{Tanh}}(\ddot{\overline{\mathbf{y}}}, \ddot{\overline{\mathbf{x}}}) \stackrel{\text{def}}{=} \ddot{\overline{\mathbf{y}}} \geq \texttt{Tanh}(\mathbf{x}^u) \wedge \ddot{\overline{\mathbf{y}}} \geq \mathbf{a}^u \ddot{\overline{\mathbf{x}}} + \mathbf{b}^u$, for the case when $\dot{\mathbf{x}}_i \leq \mathbf{x}^u_i$, we have $\ddot{\overline{\mathbf{y}}}_i \geq \tanh(\mathbf{x}^u_i) \geq \tanh(\dot{\mathbf{x}}_i)$ because $\tanh$ is monotonically increasing. For the case when $\dot{\mathbf{x}}_i > \mathbf{x}^u_i$, we have $\ddot{\overline{\mathbf{y}}}_i \geq \mathbf{a}^u_i \ddot{\mathbf{x}}_i + \mathbf{b}^u_i \geq \tanh(\dot{\mathbf{x}}_i)$ because $f(\boldsymbol{x}) = \mathbf{a}^u_i \boldsymbol{x} + \mathbf{b}^u_i$ is tangent to the concave piece of $\tanh$ in $[0, \infty]$, and $\tanh$ is monotonically increasing in $[\mathbf{x}^u_i, 0]$. Therefore, $\overline{\varphi}_{\texttt{Tanh}}(\ddot{\overline{\mathbf{y}}}, \ddot{\overline{\mathbf{x}}})$ defined in Definition C.2 implies that $\forall \mathbf{x} \in [\dot{\underline{\mathbf{x}}}, \ddot{\overline{\mathbf{x}}}].\ \texttt{Tanh}(\mathbf{x}) \leq \ddot{\overline{\mathbf{y}}}$.

**For the lower bound constraint** $\underline{\varphi}_{\texttt{Tanh}}(\dot{\underline{\mathbf{y}}}, \dot{\underline{\mathbf{x}}}) \stackrel{\text{def}}{=} \dot{\underline{\mathbf{y}}} \leq \texttt{Tanh}(\mathbf{x}^l) \wedge \dot{\underline{\mathbf{y}}} \leq \mathbf{a}^l \dot{\underline{\mathbf{x}}} + \mathbf{b}^l$, for the case when $\dot{\mathbf{x}}_i \geq \mathbf{x}^l_i$, we have $\dot{\underline{\mathbf{y}}}_i \leq \tanh(\mathbf{x}^l_i) \leq \tanh(\dot{\mathbf{x}}_i)$ because $\tanh$ is monotonically increasing. For the case when $\dot{\mathbf{x}}_i < \mathbf{x}^l_i$, we have $\dot{\underline{\mathbf{y}}} \leq \mathbf{a}^l_i \dot{\mathbf{x}}_i + \mathbf{b}^l_i \leq \tanh(\dot{\mathbf{x}}_i)$ because $f(\boldsymbol{x}) = \mathbf{a}^l_i \boldsymbol{x} + \mathbf{b}^l_i$ is tangent to the convex piece of $\tanh$ in $[-\infty, 0]$, and $\tanh$ is monotonically increasing in $[0, \mathbf{x}^l_i]$. Therefore, $\underline{\varphi}_{\texttt{Tanh}}(\ddot{\overline{\mathbf{y}}}, \ddot{\overline{\mathbf{x}}})$ defined in Definition C.2 implies that $\forall \mathbf{x} \in [\dot{\underline{\mathbf{x}}}, \ddot{\overline{\mathbf{x}}}].\ \dot{\underline{\mathbf{y}}} \leq \texttt{Tanh}(\mathbf{x})$.

□

## C.2 Parametric Linear Relaxation for Sigmoid layers

**Definition C.4.** $\mathbf{y} \stackrel{\text{def}}{=} \texttt{Sigmoid}(\mathbf{x})$ with input $\mathbf{x} \in \mathbb{R}^m$ and output $\mathbf{y} \in \mathbb{R}^m$ is defined as $\mathbf{y}_i \stackrel{\text{def}}{=} \frac{1}{1 + e^{-\mathbf{x}_i}}$.

**Definition C.5.** For Sigmoid layer with variable input bounds $[\dot{\underline{\mathbf{x}}}, \ddot{\overline{\mathbf{x}}}]$, its Parametric Linear Relaxation is defined as $\overline{\underline{\varphi}}_{\texttt{Sigmoid}}(\dot{\underline{\mathbf{y}}}, \ddot{\overline{\mathbf{y}}}, \dot{\underline{\mathbf{x}}}, \ddot{\overline{\mathbf{x}}}) \stackrel{\text{def}}{=} \overline{\varphi}_{\texttt{Sigmoid}}(\ddot{\overline{\mathbf{y}}}, \ddot{\overline{\mathbf{x}}}) \wedge \underline{\varphi}_{\texttt{Sigmoid}}(\dot{\underline{\mathbf{y}}}, \dot{\underline{\mathbf{x}}})$ where:

$$
\begin{aligned}
\overline{\varphi}_{\texttt{Sigmoid}}(\ddot{\overline{\mathbf{y}}}, \ddot{\overline{\mathbf{x}}}) &\stackrel{\text{def}}{=} \ddot{\overline{\mathbf{y}}} \geq \texttt{Sigmoid}(\mathbf{x}^u) \wedge \ddot{\overline{\mathbf{y}}} \geq \mathbf{a}^u \ddot{\overline{\mathbf{x}}} + \mathbf{b}^u \\
\underline{\varphi}_{\texttt{Sigmoid}}(\dot{\underline{\mathbf{y}}}, \dot{\underline{\mathbf{x}}}) &\stackrel{\text{def}}{=} \dot{\underline{\mathbf{y}}} \leq \texttt{Sigmoid}(\mathbf{x}^l) \wedge \dot{\underline{\mathbf{y}}} \leq \mathbf{a}^l \dot{\underline{\mathbf{x}}} + \mathbf{b}^l
\end{aligned}
\tag{22}
$$

where $\mathbf{x}^u \in \mathbb{R}^m$, $\mathbf{a}^u \in \mathbb{R}^m$, $\mathbf{b}^u \in \mathbb{R}^m$ are constants that satisfy the following two conditions: **i)** let $f(\boldsymbol{x}) = \mathbf{a}^u_i \boldsymbol{x} + \mathbf{b}^u_i$, the line defined by function $f(\boldsymbol{x})$ is tangent to the concave piece of $\texttt{sigmoid}$ at some $\boldsymbol{x} \geq 0$; **ii)** $\mathbf{a}^u \mathbf{x}^u + \mathbf{b}^u = \texttt{Sigmoid}(\mathbf{x}^u)$; viz., the line defined by function $f(\boldsymbol{x})$ intersects with convex piece of $\texttt{sigmoid}$ at $\mathbf{x}^u_i$ where $\mathbf{x}^u_i \leq 0$.

Similarly, $\mathbf{x}^l \in \mathbb{R}^m$, $\mathbf{a}^l \in \mathbb{R}^m$, $\mathbf{b}^l \in \mathbb{R}^m$ are constants that satisfy the following two conditions: **i)** let $f(\boldsymbol{x}) = \mathbf{a}^l_i \boldsymbol{x} + \mathbf{b}^l_i$, the line defined by function $f(\boldsymbol{x})$ is tangent to the convex piece of $\texttt{sigmoid}$ at some $\boldsymbol{x} \leq 0$; **ii)** $\mathbf{a}^l \mathbf{x}^l + \mathbf{b}^l = \texttt{Sigmoid}(\mathbf{x}^l)$; viz., the line defined by function $f(\boldsymbol{x})$ intersects with concave piece of $\texttt{sigmoid}$ at $\mathbf{x}^l_i$ where $\mathbf{x}^l_i \geq 0$. ■

**Theorem C.6.** *Definition C.5 is a Parametric Linear Relaxation for the Sigmoid layer.*

*Proof.* As seen in Definition C.5, $\overline{\underline{\varphi}}_{\texttt{Sigmoid}}(\dot{\underline{\mathbf{y}}}, \ddot{\overline{\mathbf{y}}}, \dot{\underline{\mathbf{x}}}, \ddot{\overline{\mathbf{x}}})$ is a linear formula. Let $m$ be the number of input dimensions of Sigmoid. $\overline{\underline{\varphi}}_{\texttt{Sigmoid}}(\dot{\underline{\mathbf{y}}}, \ddot{\overline{\mathbf{y}}}, \dot{\underline{\mathbf{x}}}, \ddot{\overline{\mathbf{x}}})$ has $4m$ linear constraints over $4m$ variables ($\dot{\underline{\mathbf{x}}} \in \mathbb{R}^m$, $\ddot{\overline{\mathbf{x}}} \in \mathbb{R}^m$, $\dot{\underline{\mathbf{y}}} \in \mathbb{R}^m$ and $\ddot{\overline{\mathbf{y}}} \in \mathbb{R}^m$). Hence, $\overline{\underline{\varphi}}_{\texttt{Sigmoid}}(\dot{\underline{\mathbf{y}}}, \ddot{\overline{\mathbf{y}}}, \dot{\underline{\mathbf{x}}}, \ddot{\overline{\mathbf{x}}})$ is a poly-size linear formula, whose size is polynomial in the number of input dimensions of the Sigmoid layer.

Now our goal is to prove that $\overline{\underline{\varphi}}_{\texttt{Sigmoid}}(\dot{\underline{\mathbf{y}}}, \ddot{\overline{\mathbf{y}}}, \dot{\underline{\mathbf{x}}}, \ddot{\overline{\mathbf{x}}})$ defined in Definition C.5 implies

$$\forall \mathbf{x} \in [\dot{\underline{\mathbf{x}}}, \ddot{\overline{\mathbf{x}}}].\ \dot{\underline{\mathbf{y}}} \leq \texttt{Sigmoid}(\mathbf{x}) \leq \ddot{\overline{\mathbf{y}}}$$

We will discuss the upper and lower bounds separately.

**For the upper bound constraint** $\overline{\varphi}_{\texttt{Sigmoid}}(\ddot{\overline{\mathbf{y}}}, \ddot{\overline{\mathbf{x}}}) \stackrel{\text{def}}{=} \ddot{\overline{\mathbf{y}}} \geq \texttt{Sigmoid}(\mathbf{x}^u) \wedge \ddot{\overline{\mathbf{y}}} \geq \mathbf{a}^u \ddot{\overline{\mathbf{x}}} + \mathbf{b}^u$, for the case when $\dot{\mathbf{x}}_i \leq \mathbf{x}^u_i$, we have $\ddot{\overline{\mathbf{y}}}_i \geq \texttt{sigmoid}(\mathbf{x}^u_i) \geq \texttt{sigmoid}(\dot{\mathbf{x}}_i)$ because $\texttt{sigmoid}$ is monotonically increasing. For the case when $\dot{\mathbf{x}}_i > \mathbf{x}^u_i$, we have $\ddot{\overline{\mathbf{y}}}_i \geq \mathbf{a}^u_i \ddot{\mathbf{x}}_i + \mathbf{b}^u_i \geq \texttt{sigmoid}(\dot{\mathbf{x}}_i)$ because

$f(\boldsymbol{x}) = \mathbf{a}_i^u \boldsymbol{x} + \mathbf{b}_i^u$ is tangent to the concave piece of `sigmoid` in $[0, \infty]$, and `sigmoid` is monotonically increasing in $[\mathbf{x}_i^u, 0]$. Therefore, $\overline{\varphi}_{\texttt{Sigmoid}}(\ddot{\mathbf{y}}, \ddot{\mathbf{x}})$ defined in Definition C.5 implies that $\forall \mathbf{x} \in [\dot{\mathbf{x}}, \ddot{\mathbf{x}}]. \texttt{Sigmoid}(\mathbf{x}) \leq \ddot{\mathbf{y}}$.

**For the lower bound constraint** $\underline{\varphi}_{\texttt{Sigmoid}}(\dot{\mathbf{y}}, \dot{\mathbf{x}}) \stackrel{\text{def}}{=} \dot{\mathbf{y}} \leq \texttt{Sigmoid}(\mathbf{x}^l) \wedge \dot{\mathbf{y}} \leq \mathbf{a}^l \dot{\mathbf{x}} + \mathbf{b}^l$, for the case when $\dot{\mathbf{x}}_i \geq \mathbf{x}_i^l$, we have $\dot{\mathbf{y}}_i \leq \texttt{sigmoid}(\mathbf{x}_i^l) \leq \texttt{sigmoid}(\dot{\mathbf{x}}_i)$ because `sigmoid` is monotonically increasing. For the case when $\dot{\mathbf{x}}_i < \mathbf{x}_i^l$, we have $\dot{\mathbf{y}}_i \leq \mathbf{a}_i^l \dot{\mathbf{x}}_i + \mathbf{b}_i^l \leq \texttt{sigmoid}(\dot{\mathbf{x}}_i)$ because $f(\boldsymbol{x}) = \mathbf{a}_i^l x + \mathbf{b}_i^l$ is tangent to the convex piece of `sigmoid` in $[-\infty, 0]$, and `sigmoid` is monotonically increasing in $[0, \mathbf{x}_i^l]$. Therefore, $\underline{\varphi}_{\texttt{Sigmoid}}(\ddot{\mathbf{y}}, \ddot{\mathbf{x}})$ defined in Definition C.5 implies that $\forall \mathbf{x} \in [\dot{\mathbf{x}}, \ddot{\mathbf{x}}]. \dot{\mathbf{y}} \leq \texttt{Sigmoid}(\mathbf{x})$. $\square$

## C.3 Parametric Linear Relaxation for ELU layers

**Definition C.7.** $\mathbf{y} \stackrel{\text{def}}{=} \texttt{ELU}(\mathbf{x})$ with input $\mathbf{x} \in \mathbb{R}^m$ and output $\mathbf{y} \in \mathbb{R}^m$ is defined as $\mathbf{y}_i \stackrel{\text{def}}{=} \texttt{elu}(\mathbf{x}_i)$ where $\texttt{elu}(\boldsymbol{x}) \stackrel{\text{def}}{=} \begin{cases} \boldsymbol{x} & \boldsymbol{x} \geq 0 \\ \alpha(e^{\boldsymbol{x}} - 1) & \boldsymbol{x} < 0 \end{cases}$ and $\alpha > 0$.

**Definition C.8.** For ELU layer with variable input bounds $[\dot{\mathbf{x}}, \ddot{\mathbf{x}}]$, its Parametric Linear Relaxation is defined as $\overline{\varphi}_{\texttt{ELU}}(\dot{\mathbf{y}}, \ddot{\mathbf{y}}, \dot{\mathbf{x}}, \ddot{\mathbf{x}}) \stackrel{\text{def}}{=} \overline{\varphi}_{\texttt{ELU}}(\ddot{\mathbf{y}}, \ddot{\mathbf{x}}) \wedge \underline{\varphi}_{\texttt{ELU}}(\dot{\mathbf{y}}, \dot{\mathbf{x}})$ where:

$$\overline{\varphi}_{\texttt{ELU}}(\ddot{\mathbf{y}}, \ddot{\mathbf{x}}) \stackrel{\text{def}}{=} \ddot{\mathbf{y}} \geq \ddot{\mathbf{x}} \wedge \ddot{\mathbf{y}} \geq 0$$
$$\underline{\varphi}_{\texttt{ELU}}(\dot{\mathbf{y}}, \dot{\mathbf{x}}) \stackrel{\text{def}}{=} \dot{\mathbf{y}} \leq \mathbf{a}\dot{\mathbf{x}} + \mathbf{b} \tag{23}$$

where $\mathbf{a} \in \mathbb{R}^m$, $\mathbf{b} \in \mathbb{R}^m$ are constants such that the line defined by $f(\boldsymbol{x}) = \mathbf{a}_i \boldsymbol{x} + \mathbf{b}_i$ for any $i$ is tangent to `elu`. $\blacksquare$

**Theorem C.9.** *Definition C.8 is a Parametric Linear Relaxation for the ELU layer.*

*Proof.* As seen in Definition C.8, $\overline{\varphi}_{\texttt{ELU}}(\dot{\mathbf{y}}, \ddot{\mathbf{y}}, \dot{\mathbf{x}}, \ddot{\mathbf{x}})$ is a linear formula. Let $m$ be the number of input dimensions of ELU. $\overline{\varphi}_{\texttt{ELU}}(\dot{\mathbf{y}}, \ddot{\mathbf{y}}, \dot{\mathbf{x}}, \ddot{\mathbf{x}})$ has $3m$ linear constraints over $4m$ variables ($\dot{\mathbf{x}} \in \mathbb{R}^m$, $\ddot{\mathbf{x}} \in \mathbb{R}^m$, $\dot{\mathbf{y}} \in \mathbb{R}^m$ and $\ddot{\mathbf{y}} \in \mathbb{R}^m$). Hence, $\overline{\varphi}_{\texttt{ELU}}(\dot{\mathbf{y}}, \ddot{\mathbf{y}}, \dot{\mathbf{x}}, \ddot{\mathbf{x}})$ is a poly-size linear formula, whose size is polynomial in the number of input dimensions of the ELU layer.

Now our goal is to prove that $\overline{\varphi}_{\texttt{ELU}}(\dot{\mathbf{y}}, \ddot{\mathbf{y}}, \dot{\mathbf{x}}, \ddot{\mathbf{x}})$ defined in Definition C.8 implies

$$\forall \mathbf{x} \in [\dot{\mathbf{x}}, \ddot{\mathbf{x}}]. \dot{\mathbf{y}} \leq \texttt{ELU}(\mathbf{x}) \leq \ddot{\mathbf{y}}$$

We will discuss the upper and lower bounds separately.

**For the upper bound constraint** $\overline{\varphi}_{\texttt{ELU}}(\ddot{\mathbf{y}}, \ddot{\mathbf{x}}) \stackrel{\text{def}}{=} \ddot{\mathbf{y}} \geq \ddot{\mathbf{x}} \wedge \ddot{\mathbf{y}} \geq 0$, for the case when $\ddot{\mathbf{x}}_i \geq 0$, by Definition C.7 we have $\ddot{\mathbf{y}}_i \geq \ddot{\mathbf{x}}_i = \texttt{elu}(\ddot{\mathbf{x}}_i)$. For the case $\ddot{\mathbf{x}}_i < 0$, because `elu` is monotonically increasing and $\texttt{elu}(0) = 0$, we have $\ddot{\mathbf{y}}_i \geq 0 \geq \texttt{elu}(\ddot{\mathbf{x}}_i)$. Therefore, $\overline{\varphi}_{\texttt{ELU}}(\ddot{\mathbf{y}}, \ddot{\mathbf{x}})$ defined in Definition C.8 implies that $\forall \mathbf{x} \in [\dot{\mathbf{x}}, \ddot{\mathbf{x}}]. \texttt{ELU}(\mathbf{x}) \leq \ddot{\mathbf{y}}$.

**For the lower bound constraint** $\underline{\varphi}_{\texttt{ELU}}(\dot{\mathbf{y}}, \dot{\mathbf{x}}) \stackrel{\text{def}}{=} \dot{\mathbf{y}} \leq \mathbf{a}\dot{\mathbf{x}} + \mathbf{b}$, because the line defined by $f(\boldsymbol{x}) = \mathbf{a}_i \boldsymbol{x} + \mathbf{b}_i$ is tangent to `elu` and `elu` is a convex function, we have $\dot{\mathbf{y}}_i \leq \mathbf{a}_i \dot{\mathbf{x}}_i + \mathbf{b}_i \leq \texttt{elu}(\dot{\mathbf{x}}_i)$. Therefore, $\underline{\varphi}_{\texttt{ELU}}(\ddot{\mathbf{y}}, \ddot{\mathbf{x}})$ defined in Definition C.8 implies that $\forall \mathbf{x} \in [\dot{\mathbf{x}}, \ddot{\mathbf{x}}]. \dot{\mathbf{y}} \leq \texttt{ELU}(\mathbf{x})$. $\square$

# D Experiment Details

## D.1 Provable fine-tuning PFT⟨·⟩

Because there are no efficient prior approaches for provable editing, we implemented *provable fine-tuning* PFT⟨·⟩ baselines that combine prior (non-provable) DNN editing approaches with a verifier in the loop: the editing stops when the verifier confirms that the DNN satisfies the property (Appendix D.1). Specifically: 1) In each epoch, provable fine-tuning first updates the DNN parameters with the DNN editing approach; 2) If the editing approach or adversarial attack cannot find any constraint violation, we call a verifier to check if all constraints are satisfied; 3) If all constraints

are satisfied, the loop terminates; otherwise the loop continues. According to the DNN architecture and editing property, we use the best verifier from (1) $\alpha,\beta$-CROWN [46, 44]: winner of the VNN Competition VNN-COMP 2021, 2022 and 2023; (2) MN-BaB [9]: runner-up of VNN-COMP 2022 and 2023; and (3) DeepT [4]: state-of-the-art (incomplete) transformer verifier.

In order to improve the performance of provable fine-tuning with a non-provable DNN editing approach, viz., preserve the DNN's accuracy, we a) freeze all batch normalization layers, b) only edit the last few layers, c) incorporate changes of parameters as regularization, and d) sample points and incorporate the changes in their outputs as regularization.

## D.2 Provable Editing on VNN Competition Benchmarks

In this section we present the further details for Section 4.1.

**Benchmarks.** We take 12 out of 15 benchmarks from VNN-COMP′22 [29], excluding **i)** "carvana_unet_2022" and "nn4sys": the support for the sigmoid activation function is not implemented in PREPARED, and is not supported by APRNN. **ii)** "vggnet16_2022": the bound computation for intermediate layers runs out of CPU and GPU memory using auto_LiRPA.

**Setup details.** For both PFT⟨DL2⟩ and PFT⟨APRNN⟩, we use DL2 to find counterexamples to edit, and we use $\alpha,\beta$-CROWN, the winner of VNN-COMP, as the verifier. We only call the verifier when DL2 cannot find counterexamples. For all approaches, we edit the last few layers, which differs for different DNNs. For PREPARED, we use auto_LiRPA or DeepPoly to compute the constant bounds for the input to the first layer to edit.

**Hyperparameters for PFT⟨DL2⟩.** SGD optimizer with a learning rate of 0.001, DL2 weight of 0.2, batch size of 32 for the **single-property** setting, and 256 for the **all-properties** setting.

**Hyperparameters for PFT⟨APRNN⟩.** For different DNNs, APRNN edits from the last six layers to the last layer. we set a bound of [-10, 10] for the change of each parameter. We use DL2 to find counterexamples for APRNN to edit in each epoch, and use the same hyperparameters as the DL2 experiments for the DL2 oracle.

**Hyperparameters for PREPARED.** For different DNNs, PREPARED edits from the last eight layers to the last layer. We use auto_LiRPA [50] or DeepPoly [39] to compute the input bounds to the first layer to edit. we set a bound of [-10, 10] for the change of each parameter.

**Results details.** In Table 4 we present the number of succeed instances for provably editing **single-properties** and **all-properties** instances in each benchmark.

## D.3 Local robustness editing for image-recognition DNNs

In this section we present the experiment details for Section 4.2.

**Property.** Given an $L^\infty$ local robustness perturbation $\varepsilon$ for an DNN $\mathcal{N}$ and a pair of input point $\mathbf{x}$ and output label $l$, the repair specification is $\forall \mathbf{x}'. \|\mathbf{x} - \mathbf{x}'\|_\infty \implies \arg\max \mathcal{N}(\mathbf{x}') = l$. In other words, for all inputs $\mathbf{x}'$ in the $L^\infty$ ball around $\mathbf{x}$ with radius $\varepsilon$, the DNN $\mathcal{N}$ should classify them as label $l$.

**Setup details.** The validation set is the same 10% of the training set, randomly selected with seed 0. For PFT⟨DL2⟩, PFT⟨SABR⟩ and PFT⟨STAPS⟩, we use MN-BaB, the complete verifier used by SABR and STAPS, as the verifier in the loop. MN-BaB is the runner-up of VNN-COMP. We only call the verifier when DL2, SABR or STAPS cannot find counterexamples. For PREPARED, we use DeepPoly to compute the constant bounds for the input to the first layer to edit.

**Hyperparameters for PFT⟨SABR⟩.** We take the base hyperparameters from the SABR training script [27]. We freeze all batch normalization layers, and search the hyperparameters: **i)** learning rate in $\{5e\text{-}4, 1e\text{-}4, 5e\text{-}5, 1e\text{-}5\}$; **ii)** batch size in $\{5, 10, 25\}$; **iii)** lambda in $\{0.1, 0.2, \ldots, 0.9\}$; **iv)** end_epoch_eps in $\{80, 60, 40, 20\}$; **v)** fine-tune the entire DNN, the last three layers, or the last layer to find the edit with the highest efficacy, then the best validation set accuracy. Other hyperparameters are the same as used for training the CIFAR10 ($\varepsilon = 2/255$) and TINYIMAGENET ($\varepsilon = 1/255$) CNN7 in the SABR paper. We use the following early-stopping criteria: **i)** CIFAR10: validation accuracy drops below 60%; **ii)** TINYIMAGENET: validation accuracy drops below 20%. The final hyperparameters for CIFAR10 are: Adam optimizer with a learning rate of 5e-4, batch size of 5,

Table 4: Comparison of the number of succeed instances for provably editing **single-instances** and **all-properties** instances in each benchmark.

(a) Comparison of the number of instances succeeded for provably editing **single-instances** instances in each benchmark.

| Benchmark | PFT⟨DL2⟩ | PFT⟨APRNN⟩ | PREPARED | Total |
|---|---|---|---|---|
| acasxu | 0 | 5 | **47** | 47 |
| rl_benchmarks | **103** | 84 | **103** | 103 |
| tllverifybench | 20 | 5 | **21** | 21 |
| cifar100_tinyimagenet_resnet | 0 | 0 | **29** | 29 |
| sri_resnet_a | 8 | 2 | **52** | 52 |
| sri_resnet_b | 8 | 0 | **44** | 44 |
| reach_prob_density | 0 | 0 | **14** | 14 |
| mnist_fc | 1 | 1 | **22** | 22 |
| cifar_biasfield | 1 | 0 | **4** | 4 |
| cifar2020 | 35 | 0 | **55** | 55 |
| collins_rul_cnn | 8 | 20 | **27** | 27 |
| oval21 | 1 | 0 | **5** | 5 |
| all | 185 | 117 | **423** | 423 |

(b) Comparison of the number of instances succeeded for provably editing **all-properties** instances in each benchmark.

| Benchmark | PFT⟨DL2⟩ | PFT⟨APRNN⟩ | PREPARED | Total |
|---|---|---|---|---|
| acasxu | 38 | 40 | **42** | 42 |
| rl_benchmarks | 0 | 1 | **2** | 3 |
| tllverifybench* | 0 | 0 | 0 | 0 |
| cifar100_tinyimagenet_resnet | 0 | 0 | 5 | 5 |
| sri_resnet_a | 0 | **1** | **1** | 1 |
| sri_resnet_b | 0 | 0 | **1** | 1 |
| reach_prob_density | 2 | 1 | **3** | 3 |
| mnist_fc | **3** | 2 | **3** | 3 |
| cifar_biasfield* | 0 | 0 | 0 | 0 |
| cifar2020 | 0 | 1 | **3** | 3 |
| collins_rul_cnn | 0 | 0 | 0 | 3 |
| oval21 | **2** | **2** | **2** | 2 |
| all | 45 | 48 | **62** | 66 |

\* The DNNs that violate the properties in these benchmarks only have one property, hence are subsumed in the prior single-property setting and excluded in this all-properties setting.

lambda of 0.1, end_epoch_eps of 80, and fine-tune the last three layers. The final hyperparameters for TINYIMAGENET are: Adam optimizer with a learning rate of 5e-4, batch size of 5, lambda of 0.1, end_epoch_eps of 80, and fine-tune the last three layers.

**Hyperparameters for** PFT⟨STAPS⟩. We take the base hyperparameters from the STAPS training script in the TAPS code [25]. We freeze all batch normalization layers, and search the following hyperparameters: **i)** learning rate in $\{5e\text{-}4, 1e\text{-}4, 5e\text{-}5, 1e\text{-}5\}$; **ii)** batch size in $\{5, 10, 25\}$; **iii)** reg_lambda in $\{0.1, 0.2, \ldots, 0.9\}$; **iv)** end_epoch_eps in $\{80, 60, 40, 20\}$. **v)** fine-tune the entire DNN, the last three layers, or the last layer. to find the edit with the highest efficacy, then the best validation set accuracy. Other hyperparameters are the same as used for training CIFAR10 ($\varepsilon = {}^2/255$) and TINYIMAGENET ($\varepsilon = {}^1/255$) CNN7 using STAPS in the TAPS paper We use the following early-stopping criteria: **i)** CIFAR10: validation accuracy drops below 60%; **ii)** TINYIMAGENET: validation accuracy drops

below 20%. The final hyperparameters for CIFAR10 are: Adam optimizer with a learning rate of 1e-5, batch size of 10, reg_lambda of 0.9, end_epoch_eps of 80, and fine-tune the last three layers. The final hyperparameters for TINYIMAGENET are: Adam optimizer with a Learning rate of 1e-5, batch size of 10, reg_lambda of 0.8, end_epoch_eps of 80, and fine-tune the last three layers.

**Hyperparameters for** PFT⟨DL2⟩**.** We take the base hyperparameters from the RobustnessG training script in the DL2 code [11]. We freeze all the batch normalization layers, and search for the following hyperparameters: **i)** learning rate in {1e-3, 5e-4, 1e-4, 5e-5}; **ii)** batch size in {5, 10, 25}; **iii)** fine-tune the entire DNN, the last layer or the last three layers. to find the edit with the highest efficacy, then the best validation set accuracy. We use the following early-stopping criteria: **i)** CIFAR10: validation accuracy drops below 70%; **ii)** TINYIMAGENET: validation accuracy drops below 20%. The final hyperparameters for CIFAR10 are: SGD optimizer with learning rate of 1e-4, batch size of 10, DL2 weight of 0.2, fine-tune the last three layers. The final hyperparameters for TINYIMAGENET are: SGD optimizer with learning rate of 1e-4, batch size of 10, DL2 weight of 0.2, fine-tune the last three layers.

**Hyperparameters for** APRNN**.** We edit up to the fifth last layer, and set a bound [-10, 10] for the change of each DNN parameter. For both CIFAR10 and TINYIMAGENET, the final result comes from editing the last layer.

**Hyperparameters for** PREPARED**.** We edit up to the fifth last layer, and set a bound [-10, 10] for the change of each DNN parameter. For both CIFAR10 and TINYIMAGENET, the final result comes from editing the last layer.

## D.4 Local robustness editing for sentiment classification BERT transformers

In this section we present the experiment details for Section 4.3.

**Setup details.** We use the default validation set used in the DeepT training scripts [3]. For PFT⟨DL2⟩, we use DeepT as the verifier in the loop. We only call the verifier when DL2 cannot find counterexamples. For PREPARED, we use DeepT to compute the constant bounds for the input to the first layer to edit.

**Hyperparameters for** PFT⟨DL2⟩**.** We incorporate the default training script and hyperparameters from DeepT [4]. We search the hyperparameters **i)** fine-tune the last three layers or only the last layer; **ii)** batch size in {4, 8, 16, 32}; **iii)** learning rate in {1e-3, 5e-3, 1e-4, 5e-4}; to find the edit with the best efficacy, then the best validation accuracy. For $\varepsilon$=1e-4, the final hyperparameters are: fine-tune the last layer, batch size of 32, Adam optimizer with learning rate of 1e-4, DL2 weight of 0.1 For $\varepsilon$=5e-4, the final hyperparameters are: fine-tune the last layer, batch size of 16, Adam optimizer with learning rate of 1e-4, DL2 weight of 0.1.

**Hyperparameters for** PFT⟨APRNN⟩**.** We edit up to the third last layer, and set a bound [-10, 10] for the change of each DNN parameter. The final result comes from editing the last layer.

**Hyperparameters for** PREPARED**.** We edit up to the third last layer, and set a bound [-10, 10] for the change of each DNN parameter. The final result comes from editing the last layer.

## D.5 Provable training for physics-plausible DNNs

In this section we present the experiment details for Section 4.4.

**Hyperparameters for** DL2**.** We search the hyperparameters **i)** batch size in {4, 8, 16, 32}; **ii)** learning rate in {1e-3, 5e-3, 1e-4, 5e-4}; We train 2000 epochs and take the epoch with the highest validation accuracy. The final hyperparameters are: batch size of 8, SGD optimizer with lr of 1e-3, DL2 weight of 0.2, 1241 epochs.

**Hyperparameters for** GD**.** We search the hyperparameters **i)** batch size in {4, 8, 16, 32}; **ii)** learning rate in {1e-3, 5e-3, 1e-4, 5e-4}; We train 2000 epochs and take the epoch with the highest validation accuracy. The final hyperparameters are: batch size of 32, SGD optimizer with lr of 1e-3, 635 epochs.

**Hyperparameters for** GD⟨APRNN⟩**.** We edit up to the first layer, and set a bound [-10, 10] for the change of each DNN parameter. The final result comes from editing the last layer.

**Hyperparameters for** `GD`⟨`PREPARED`⟩**.**   We edit up to the first layer, and set a bound [-10, 10] for the change of each DNN parameter. We use `DeepPoly` to compute the input bounds to the first layer to edit. The final result comes from editing the last three layers.

